# How Does Black-Box Impact the Learning Guarantee of Stochastic Compositional Optimization?

**Jun Chen**
College of Informatics, Huazhong Agricultural University, China
`cj850487243@163.com`

**Hong Chen**[*]
College of Informatics, Huazhong Agricultural University, China
Engineering Research Center of Intelligent Technology for Agriculture, China
`chenh@mail.hzau.edu.cn`

**Bin Gu**
School of Artificial Intelligence, Jilin University, China
Mohamed bin Zayed University of Artificial Intelligence
`jsgubin@gmail.com`

## Abstract

Stochastic compositional optimization (SCO) problem constitutes a class of optimization problems characterized by the objective function with a compositional form, including the tasks with known derivatives, such as AUC maximization, and the derivative-free tasks exemplified by black-box vertical federated learning (VFL). From the learning theory perspective, the learning guarantees of SCO algorithms with known derivatives have been studied in the literature. However, the potential impacts of the derivative-free setting on the learning guarantees of SCO remains unclear and merits further investigation. This paper aims to reveal the impacts by developing a theoretical analysis for two derivative-free algorithms, black-box SCGD and SCSC. Specifically, we first provide the sharper generalization upper bounds of convex SCGD and SCSC based on a new stability analysis framework more effective than prior work under some milder conditions, which is further developed to the non-convex case using the almost co-coercivity property of smooth function. Then, we derive the learning guarantees of three black-box variants of non-convex SCGD and SCSC with additional optimization analysis. Comparing these results, we theoretically uncover the impacts that a better gradient estimation brings a tighter learning guarantee and a larger proportion of unknown gradients may lead to a stronger dependence on the gradient estimation quality. Finally, our analysis is applied to two SCO algorithms, FOO-based vertical VFL and VFL-CZOFO, to build the first learning guarantees for VFL that align with the findings of SCGD and SCSC.

## 1 Introduction

In recent years, stochastic compositional optimization (SCO), a class of optimization methods that incorporate the compositional form $f(g(w))$, has garnered significant attention in the research

---

[*]Corresponding author.

community [1, 2, 3, 4, 5, 6, 7]. It represents a special case of stochastic bilevel optimization [8]

$$\min_{w\in\mathcal{W}\in\mathbb{R}^p} \mathbb{E}_{\bar{z}}\left[f_{\bar{z}}\left(w, v^*(w)\right)\right] \quad \text{s.t.} \quad v^*(w) = \arg\min_{v\in\mathbb{R}^d} \mathbb{E}_z\left[h_z(w, v)\right], \tag{1}$$

where $\mathbb{E}_z[\cdot]$ represents the expectation with respect to (w.r.t.) the sample $z$, parameters $w \in \mathcal{W} \in \mathbb{R}^p$ and $v \in \mathbb{R}^d$. If the inner function $h_z(w, v) = \|v - g_z(w)\|^2$ and the outer function $f$ is only a function of $v$, i.e., $f_{\bar{z}}(w, v) = f_{\bar{z}}(v)$, the stochastic bilevel optimization reduces to the SCO:

$$\min_{w\in\mathcal{W}\in\mathbb{R}^p} \left\{F(w) = f(g(w)) = \mathbb{E}_{\bar{z}}\left[f_{\bar{z}}(\mathbb{E}_z\left[g_z(w)\right])\right]\right\}, \tag{2}$$

where $F(w)$ is the compositional population risk, $f : \mathbb{R}^d \to \mathbb{R}$, $g : \mathbb{R}^p \to \mathbb{R}^d$, $f(v) = \mathbb{E}_{\bar{z}}\left[f_{\bar{z}}(v)\right]$ and $g(w) = \mathbb{E}_z\left[g_z(w)\right]$.

Many applications adhere to the form of SCO (2) such as risk averse optimization [9], group distributionally robust optimization [10], AUC maximization [11, 12, 13, 14], model-agnostic meta-learning [15], and first-order-optimization-based vertical federated learning (FOO-based VFL) [16]. Apart from the above applications with available derivatives, there exist derivative-free scenarios as well, such as reinforcement learning [17] and zeroth-order-optimization-based (ZOO-based) VFL [18, 19]. [2] discussed the extension of stochastic compositional gradient descent (SCGD) to the derivative-free setting, called black-box SCGD, where the zeroth-order information of $f$ or $g$ is available through sampling.

From the perspective of statistical learning theory [20], the theoretical guarantees pertaining to generalization and optimization performance for SCO algorithms are worth studying to validate their empirical behaviors. The former assesses the disparity between the empirical performance and the population performance for the trained model. The latter measures the empirical performance gap between the trained model and the empirical optimal model. To our knowledge, there is only one study attempting to provide a generalization guarantee in this area. [21] has pioneered the generalization understanding of two notable SCO algorithms, i.e., SCGD and SCSC [5] via algorithmic stability tool. They have achieved satisfactory excess risk bounds by selecting some specific values of $T$ to balance stability results and optimization errors. However, the intricacy of their analysis framework brings some unnecessary terms in their results leading to these so large $T$ values that it is practically challenging to complete these iterations within a reasonable timeframe. Furthermore, the study has yet to consider a more practical scenario beyond convex and strongly convex cases, specifically, the non-convex case. For the optimization guarantee, plenty of work devotes to studying the convergence behaviors of some first-order SCO algorithms [2, 3, 4, 5] and their extensions [22, 23, 24, 25]. For example, [2] proved that SCGD can converge almost surely to an existing optimal solution with the rate $\mathcal{O}\left(T^{-1/4}\right)$ for non-smooth convex problems and the rate $\mathcal{O}\left(T^{-2/7}\right)$ for smooth convex problems, where $T$ is the total number of iterations. [5] presented that stochastically corrected stochastic compositional gradient method (SCSC) can achieve the same convergence rate $\mathcal{O}\left(T^{-1/2}\right)$ as SGD for non-compositional problems. However, there exists a research gap in the optimization analysis for the derivative-free SCO algorithm as well as in its generalization analysis.

Considering these problems, this paper leverages algorithmic stability to obtain some similar and even superior results of SCGD and SCSC under the milder parameter selection and the non-convex condition. More importantly, to apply a broader class of stochastic optimization problems, this paper pioneers the theoretical analysis of the black-box SCO algorithms, which uncovers the impacts of black-box on the learning guarantees of SCO algorithms. Our main contributions are listed as follows.

- *Generalization guarantees under some milder settings.* Firstly, we provide the sharper generalization upper bounds of convex SCGD and SCSC based on a new stability analysis framework more effective than prior work [21] with a more practical selection of $T$. Subsequently, we develop the above convex analysis to the non-convex case by introducing the almost co-coercivity property of smooth function, which yields satisfactory generalization guarantees of SCGD and SCSC under the non-convex condition.

- *Learning guarantees for black-box SCO algorithms.* To apply a broader class of stochastic optimization problems, we further consider three black-box variants (outer, inner, and full black-box) of SCGD and SCSC to obtain the generalization and optimization upper bounds similar to the ones of SCGD and SCSC. Comparing the first-order and zeroth-order results, several key insights into the impacts of black-box on the learning guarantees of SCO algorithms are shown: 1) a closer estimation distance brings a better result; 2) more

estimation directions lead to a better result; 3) a larger proportion of unknown gradients results in a stronger dependence on the gradient estimation quality.

- *Applications on VFL.* Finally, we explore the applications of our analysis framework to two specific SCO algorithms, i.e., FOO-based VFL and VFL-CZOFO, where we build the pioneering stability-based generalization and optimization guarantees for first-order and zeroth-order VFL algorithms that align with the findings of SCGD and SCSC.

## 2 Preliminaries

This section describes the learning paradigm of SCO and introduces two popular SCO algorithms (SCGD and SCSC) and their black-box variants in detail at first. Then, some necessary definitions and assumptions are provided for our theoretical analysis. The explanations for all symbols are shown in Table 3 located in *Appendix A*.

Considering a stochastic compositional optimization algorithm (2), the distributions of sample $z$ and $\bar{z}$ are unknown. The training dataset $S = S^z \cup S^{\bar{z}} = \{z_1, ..., z_n\} \cup \{\bar{z}_1, ..., \bar{z}_m\}$ is available to obtain the final output model parameter $A(S)$ via minimizing the following compositional empirical risk

$$F_S(w) = f_S(g_S(w)) = \frac{1}{m} \sum_{j=1}^{m} f_{\bar{z}_j} \left( \frac{1}{n} \sum_{i=1}^{n} g_{z_i}(w) \right),$$

where $g_S(w) = \frac{1}{n} \sum_{i=1}^{n} g_{z_i}(w)$, $f_S(v) = \frac{1}{m} \sum_{j=1}^{m} f_{\bar{z}_j}(v)$, $z_1, ..., z_n, \bar{z}_1, ..., \bar{z}_m$ are independent. Besides, we denote by $w(S), w^*$ the empirical optimal model parameter on $S$ and the global optimal model parameter, defined as $w(S) = \arg \min_{w \in \mathcal{W}} F_S(w)$ and $w^* = \arg \min_{w \in \mathcal{W}} F(w)$. Then, the generalization error, optimization error and excess risk of $A(S)$ are given by $|F(A(S)) - F_S(A(S))|$, $F_S(A(S)) - F_S(w(S))$ and $F(A(S)) - F(w^*)$, respectively. Since $\mathbb{E}[F(A(S)) - F(w^*)] \geq 0$ and $\mathbb{E}[F_S(w(S)) - F(w^*)] \leq 0$, the excess risk of $A(S)$ can be decomposed as the summation of generalization error and optimization error as follows

$$\mathbb{E}[F(A(S)) - F(w^*)] \leq \mathbb{E}[|F(A(S)) - F_S(A(S))|] + \mathbb{E}[F_S(A(S)) - F_S(w(S))], \quad (3)$$

where $\mathbb{E}[\cdot]$ denotes the expectation w.r.t. all randomness.

In this work, we primarily investigate the learning guarantees of two prevalent SCO algorithms, i.e., SCGD [2] and SCSC [5], along with their black-box variants. Algorithm 1 presents the detailed parameter update procedures of these algorithms. The difference between SCGD and SCSC lies in the update of the outer model parameter $v_t$. For SCGD, $v_{t+1}$ is the linear combination of $v_t$ and $g_{z_{i_t}}(w_t)$. However, this update may lead to a suboptimal convergence rate when the learning rate $\beta$ of the outer model update is smaller than the learning rate $\eta_t$ utilized for the inner model update. To alleviate this problem, SCSC updates $v_{t+1}$ with the combination of the "corrected" $v_t$ and $g_{z_{i_t}}(w_t)$, where $v_t$ is corrected by $g_{z_{i_t}}(w_t) - g_{z_{i_t}}(w_{t-1})$ so that $v_{t+1}$ approximates $g_{z_{i_t}}(w_t)$ [5]. Besides, [2] discussed the extension of SCGD to the derivative-free setting where only the zeroth-order information of $g$ or $f$ is available through sampling, which potentially applies to a broader class of stochastic optimization problems. Here, we show the first-order gradient estimation of $f$, which is similar to that of $g$. The unknown first-order gradient of $f$ is estimated by Equation (4) and then approximated by Taylor expansion (5) in our analysis

$$\tilde{\nabla} f_{\bar{z}_{j_t}}(v_{t+1}) = \frac{1}{b} \sum_{l=1}^{b} \frac{u_{t,l}}{\mu} \left( f_{\bar{z}_{j_t}}(v_{t+1} + \mu u_{t,l}) - f_{\bar{z}_{j_t}}(v_{t+1}) \right) \quad (4)$$

$$= \frac{1}{b} \sum_{l=1}^{b} \left( \langle \nabla f_{\bar{z}_{j_t}}(v_{t+1}), u_{t,l} \rangle u_{t,l} + \left( \frac{\mu}{2} (u_{t,l})^\top \nabla^2 f_{\bar{z}_{j_t}}(v)|_{v=v_{t+1}^*} u_{t,l} \right) u_{t,l} \right), \quad (5)$$

where $\{u_{t,l}\}_{l=1}^{b}$ is the set of independent and identically distributed (i.i.d.) random direction vectors (obeying the $d$-dimensional uniform distribution), and $\mu$ is the distance between two model parameters ($v_{t+1} + \mu u_{t,l}$ and $v_{t+1}$) used to estimate the gradient in the $l$-th direction.

Drawing inspiration from the classical non-compositional stability analysis work [26] and the pioneering work [21] investigating the generalization of SCO, we introduce the definition of uniform model stability as follows.

---

**Algorithm 1** (Black-box) SCGD / SCSC

---

**Require:** $v_1, w_1$: initial outer model and inner models; $\beta, \eta_1$: initial learning rates

    **for all** $t = 1, ..., T - 1$ **do**

        Randomly sample $i_t \in [n]$, obtain $g_{z_{i_t}}(w_t)$ and $\nabla g_{z_{i_t}}(w_t)$ (**Inner black-box:** obtain $\tilde{\nabla} g_{z_{i_t}}(w_t)$ similar to Equation (4))

        **SCGD:** Update $v_{t+1} = (1 - \beta)v_t + \beta g_{z_{i_t}}(w_t)$

        **SCSC:** Update $v_{t+1} = (1 - \beta)v_t + \beta g_{z_{i_t}}(w_t) + (1 - \beta)(g_{z_{i_t}}(w_t) - g_{z_{i_t}}(w_{t-1}))$

        Randomly sample $j_t \in [m]$, obtain $\nabla f_{\bar{z}_{j_t}}(v_{t+1})$ (**Outer black-box:** obtain $\tilde{\nabla} f_{\bar{z}_{j_t}}(v_{t+1})$)

        Update $w_{t+1} = w_t - \eta_t \nabla g_{z_{i_t}}(w_t) \nabla f_{\bar{z}_{j_t}}(v_{t+1}) / w_{t+1} = w_t - \eta_t \nabla g_{z_{i_t}}(w_t) \tilde{\nabla} f_{\bar{z}_{j_t}}(v_{t+1})$

        $/ w_{t+1} = w_t - \eta_t \tilde{\nabla} g_{z_{i_t}}(w_t) \nabla f_{\bar{z}_{j_t}}(v_{t+1}) / w_{t+1} = w_t - \eta_t \tilde{\nabla} g_{z_{i_t}}(w_t) \tilde{\nabla} f_{\bar{z}_{j_t}}(v_{t+1})$

    **end for**

**Ensure:** Final model $w_T$

---

**Definition 1.** *The randomized algorithm A for SCO problem is uniformly model $(\epsilon_z, \epsilon_{\bar{z}})$-stable if*

$$\mathbb{E}_A \left[ \|A(S) - A(S^{i,z})\| \right] \leq \epsilon_z \text{ and } \mathbb{E}_A \left[ \|A(S) - A(S^{j,\bar{z}})\| \right] \leq \epsilon_{\bar{z}},$$

*where $\| \cdot \|$ is the Euclidean distance $\| \cdot \|_2$ and $S = \{z_1, ..., z_n, \bar{z}_1, ..., \bar{z}_m\}$, $S^{i,z} = \{z_1, ..., z_{i-1}, z_i', z_{i+1}, ..., z_n, \bar{z}_1, ..., \bar{z}_m\}$, $S^{j,\bar{z}} = \{z_1, ..., z_n, \bar{z}_1, ..., \bar{z}_{j-1}, \bar{z}_j', \bar{z}_{j+1}, ..., \bar{z}_m\}$ for any $i \in [n], j \in [m]$.*

According to the foundational concept of algorithmic stability, Definition 1 considers the two datasets obtained from the perturbation of a single sample in $\{z_i\}_{i+1}^n$ and $\{\bar{z}_j\}_{j=1}^m$ respectively, where the altered sample $z_i'$ is i.i.d. to $z_i$, and so does $\bar{z}_j'$. Prior to filling the relationship gap between the uniformly model stability and the generalization error $\mathbb{E}[|F(A(S)) - F_S(A(S))|]$, it is essential to make some fundamental assumptions, i.e., Lipschitz continuity (bounded first-order gradient) of $g, f$ and bounded variance of $g$.

**Assumption 1.** *For any parameters $w, w' \in \mathcal{W}, v, v' \in \mathbb{R}^d$ and some $L_g, L_f > 0$, functions $g_z(w)$ and $f_{\bar{z}}(v)$ are Lipschitz continuous, i.e., $\|\nabla g_z(w)\| \leq L_g$ and $\|\nabla f_{\bar{z}}(v)\| \leq L_f$, which also mean that $\|g_z(w) - g_z(w')\| \leq L_g \|w - w'\|$ and $|f_{\bar{z}}(v) - f_{\bar{z}}(v')| \leq L_f \|v - v'\|$.*

In numerous compositional [2, 5, 6, 7] and non-compositional studies [26, 27], Assumption 1 serves as a general theoretical bridge analyzing the generalization and optimization performance.

**Assumption 2.** *For any $w \in \mathcal{W}$ and some $V_g > 0$, the variance of function $g_z(w)$ is upper bounded by $V_g$, i.e., $\mathbb{E}_z \left[ \|g_z(w) - g(w)\|^2 \right] \leq V_g$.*

The bounded variance is also a classical condition for statistical learning theory [1, 2, 3, 5, 6, 7, 28, 29, 30] which limits the ranges of the variance value of the given functions $g$. Utilizing these two fundamental assumptions, Theorem 1 builds a rigorous relationship between stability and generalization error, thereby enabling stability to measure the generalization performance in the subsequent analysis. Note that, Theorem 1 was previously proved by [21] (Theorem 2.3), so we omit its detailed proof here for brevity.

**Theorem 1.** *[21] Let Assumptions 1, 2 hold. Assume the randomized algorithms A for SCO problem is uniformly model $(\epsilon_z, \epsilon_{\bar{z}})$-stable, then,*

$$\mathbb{E}[|F(A(S)) - F_S(A(S))|] \leq L_g L_f (4\epsilon_z + \epsilon_{\bar{z}}) + L_f \sqrt{n^{-1} V_g}.$$

**Remark 1.** *As mentioned in [21], Theorem 1 is the compositional counterpart of Theorem 2.2 in [26]. In other words, the above result is equivalent to $\mathbb{E}[|F(A(S)) - F_S(A(S))|] \leq L_f \epsilon_{\bar{z}}$ when $g_z(w) = w$, i.e., $F(w) = \mathbb{E}_{\bar{z}}[f_{\bar{z}}(w)]$ and $F_S(w) = \frac{1}{m} \sum_{j=1}^m f_{\bar{z}_j}(w)$. If the order of $\epsilon_{\bar{z}}$ is faster than $\mathcal{O}\left(\epsilon_z + n^{-\frac{1}{2}}\right)$, the generalization upper bound of SCO algorithm will be primarily constrained by the term $4L_g L_f \epsilon_z + L_f \sqrt{n^{-1} V_g}$ attributed to the compositional structure. Otherwise, there is little difference between Theorem 1 and Theorem 2.2 [26].*

The subsequent assumptions and definition are required by the stability analysis in Section 3.

**Assumption 3.** *For any parameters $w, w' \in \mathcal{W}, v, v' \in \mathbb{R}^d$ and some $\alpha_g, \alpha_f, \alpha > 0$, functions $g_z(w)$, $f_{\bar{z}}(v)$ and $f_{\bar{z}}(g_z(w))$ are smooth, i.e., $\|\nabla^2 g_z(w)\| \leq \alpha_g$, $\|\nabla^2 f_{\bar{z}}(v)\| \leq \alpha_f$ and $\|\nabla^2 f_{\bar{z}}(g_z(w))\| \leq \alpha$, which also mean that*

$$\|\nabla g_z(w) - \nabla g_z(w')\| \leq \alpha_g \|w - w'\|, \quad \|\nabla f_{\bar{z}}(v) - \nabla f_{\bar{z}}(v')\| \leq \alpha_f \|v - v'\|$$

*and*

$$\|\nabla f_{\bar{z}}(g_z(w)) - \nabla f_{\bar{z}}(g_z(w'))\| \leq \alpha \|w - w'\|.$$

**Definition 2.** *For any parameter $v, v' \in \mathbb{R}^d$, a function $f : \mathbb{R}^d \to \mathbb{R}$ is convex if $f(v) \geq f(v') + \langle \nabla f(v'), v - v' \rangle$.*

**Assumption 4.** *For any $w \in \mathcal{W}, v \in \mathbb{R}^d$, direction vector $u$, step size $\mu > 0$ and some $M_g, M_f, M'_g, M'_f > 0$, the following inequalities hold*

$$\|g_z(w + \mu u) - g_z(w)\| \leq M_g, \quad |f_{\bar{z}}(v + \mu u) - f_{\bar{z}}(v)| \leq M_f,$$

*and*

$$\|\nabla g_z(w + \mu u) - \nabla g_z(w)\| \leq M'_g, \quad \|\nabla f_{\bar{z}}(v + \mu u) - \nabla f_{\bar{z}}(v)\| \leq M'_f.$$

**Remark 2.** *Assumption 3 is the most important condition for our analysis since there are several key properties (Lemma 4) of smoothness required to measure the algorithmic stability. In addition, our stability analysis framework relies crucially on another key lemma (called co-coercive lemma, Lemma 1) derived from the smoothness and convexity of the function $f(w)$. Therefore, we provide the definition of convexity in Definition 2. Except for the convex case (Section 3.1), this work mainly considers some non-convex cases (Sections 3.1, 3.2). Although the co-coercive lemma is not available without the convexity condition, a surrogate lemma (called almost co-coercive lemma, Lemma 3) takes a similar role within our analysis framework. Finally, Assumption 4 gives the upper bounds of the difference between two adjacent function values for $g, f, \nabla g, \nabla f$, which represents a less stringent assumption compared to the general bounded condition [26]. Specifically, Assumption 4 is different from the assumption $|f| \leq M$. Assumption 4 requires the distance between two adjacent function outputs to be bounded, i.e., $\|g(w + \mu u) - g(w)\| \leq M_g, |f(v + \mu u) - f(v)| \leq M_f$, which is milder than $|f| \leq M$. Besides, it also requires the distance between two adjacent gradient outputs to be bounded, i.e., $\|\nabla g(w + \mu u) - \nabla g(w)\| \leq M'_g, \|\nabla f(v + \mu u) - \nabla f(v)\| \leq M'_f$, which is milder than bounded gradient condition $\|\nabla f\| \leq L$ [26].*

## 3 Main Results

This section presents the learning guarantees of two SCO algorithms (SCGD and SCSC) under several cases. The comparisons among our results and previous work are summarized in Tables 1, 2, and their proofs are provided in *Appendices C, D*.

### 3.1 Learning Guarantees for General SCO

Firstly, we consider the generalization analysis for the general convex SCO algorithm.

**Theorem 2.** *Let Assumptions 1, 3 hold and the function $f(g(w))$ is convex. Assume that the randomized algorithms $A$ (Algorithm 1) for SCO problem brings the model sequences $\{w_t\}_{t=1}^T$ and $\left\{w_t^{i,z}\right\}_{t=1}^T \left(\left\{w_t^{j,\bar{z}}\right\}_{t=1}^T\right)$ on $S$ and $S^{i,z}$ $(S^{j,\bar{z}})$ with the step size sequence $\{\eta_t\}_{t=1}^T$.*

*(a) For SCGD with $\eta_t \leq \frac{2\beta}{\alpha t}$, the final output $A(S) = w_T$ is uniformly model $(\epsilon_z, \epsilon_{\bar{z}})$-stable with*

$$\epsilon_z = \frac{4 L_g L_f \beta \log(eT)}{\alpha n} \text{ and } \epsilon_{\bar{z}} = \frac{4 L_g L_f \beta \log(eT)}{\alpha m}.$$

*(b) For SCSC with $\eta_t \leq \frac{2}{\alpha t}$, the final output $A(S) = w_T$ is uniformly model $(\epsilon_z, \epsilon_{\bar{z}})$-stable with*

$$\epsilon_z = \frac{4 L_g L_f \log(eT)}{\alpha n} \text{ and } \epsilon_{\bar{z}} = \frac{4 L_g L_f \log(eT)}{\alpha m}.$$

Table 1: Comparisons among the stability-based generalization guarantees for SCO algorithms and SGD (Thm.-Theorem; Cor.-Corollary; $*$-high probability bound; $L, \alpha, V, M$,C.-Lipschitz continuity, smoothness, bounded variance, bounded function, and convexity assumptions; $c$-a positive constant; $\sqrt{}$-has such a property; $\times$-hasn't such a property).

| Algorithm | Generalization | Assumptions | | | | |
|---|---|---|---|---|---|---|
| | | $L$ | $\alpha$ | $V$ | $M$ | C. |
| SGD ([26] Thm. 3.8) | $\mathcal{O}\left(n^{-1}\log T\right)$ | $\sqrt{}$ | $\sqrt{}$ | $\times$ | $\times$ | $\sqrt{}$ |
| SGD ([31] Thm. 4) | $*\mathcal{O}\left(\left(n^{-1}\sqrt{T}+n^{-\frac{1}{2}}\right)\log n\right)$ | $\sqrt{}$ | $\sqrt{}$ | $\times$ | $\sqrt{}$ | $\sqrt{}$ |
| SCGD/SCSC ([21] Thm. 3.7) | $\mathcal{O}\left(T^{-\frac{1}{7}}+\left(n^{-1}+m^{-1}\right)T^{\frac{1}{7}}+m^{-\frac{1}{2}}\right)$ | $\sqrt{}$ | $\sqrt{}$ | $\sqrt{}$ | $\sqrt{}$ | $\sqrt{}$ |
| SCGD/SCSC (Thm. 2) | $\mathcal{O}\left(\left(n^{-1}+m^{-1}\right)\log T+n^{-\frac{1}{2}}\right)$ | $\sqrt{}$ | $\sqrt{}$ | $\sqrt{}$ | $\sqrt{}$ | $\sqrt{}$ |
| SGD ([26] Thm. 3.12) | $\mathcal{O}\left(n^{-1}T^{\frac{\alpha c}{\alpha c+1}}\right)$ | $\sqrt{}$ | $\sqrt{}$ | $\times$ | $\times$ | $\times$ |
| SGD ([32] Thm. 15) | $\mathcal{O}\left(n^{-1}T^{\frac{\alpha c}{\alpha c+1}}\right)$ | $\sqrt{}$ | $\sqrt{}$ | $\times$ | $\sqrt{}$ | $\times$ |
| SGD ([30] Thm. 1) | $\mathcal{O}\left(n^{-1}\log T\right)$ | $\sqrt{}$ | $\sqrt{}$ | $\sqrt{}$ | $\sqrt{}$ | $\times$ |
| SGD ([33] Cor. 17) | $\mathcal{O}\left(n^{-1}T\right)$ | $\times$ | $\times$ | $\sqrt{}$ | $\sqrt{}$ | $\times$ |
| SCGD/SCSC (Thm. 3, 4, Cor. 4, 2) | $\mathcal{O}\left(\left(n^{-1}+m^{-1}\right)T^{\frac{1}{2}}\log T+n^{-\frac{1}{2}}\right)$ | $\sqrt{}$ | $\sqrt{}$ | $\sqrt{}$ | $\sqrt{}$ | $\times$ |
| VFL (Cor. 4, 5) | $\mathcal{O}\left(n^{-1}T^{\frac{1}{2}}\log T\right)$ | $\sqrt{}$ | $\sqrt{}$ | $\sqrt{}$ | $\sqrt{}$ | $\times$ |

**Remark 3.** *Based on a new stability analysis framework more effective than prior work [21], Theorem 2 states the stability upper bounds $\mathcal{O}\left(\left(n^{-1}+m^{-1}\right)\beta\log T\right)$ for SCGD and $\mathcal{O}\left(\left(n^{-1}+m^{-1}\right)\log T\right)$ for SCSC under the convex condition, which derives a generalization bound $\mathcal{O}\left(\left(n^{-1}+m^{-1}\right)\log T+n^{-\frac{1}{2}}\right)$ by combining with Theorem 1. Previously, [21] provided the stability results $\mathcal{O}\left(\eta T\left(n^{-1}+m^{-1}\right)+\eta\left(T^{\frac{1}{2}}+\beta^{-\frac{c}{2}}T^{-\frac{c}{2}+1}+\beta^{\frac{1}{2}}T\right)+\eta^2\beta^{-1}T\right)$ for SCGD and $\mathcal{O}\left(\eta T\left(n^{-1}+m^{-1}\right)+\eta\left(T^{\frac{1}{2}}+\beta^{-\frac{c}{2}}T^{-\frac{c}{2}+1}+\beta^{\frac{1}{2}}T\right)+\eta^2\beta^{-\frac{1}{2}}T\right)$ for SCSC in the same setting, where $c>0$ is an arbitrary constant. They selected $\eta=\mathcal{O}\left(T^{-\frac{6}{7}}\right)$, $\beta=\mathcal{O}\left(T^{-\frac{4}{7}}\right), c>2$, $T=\mathcal{O}\left(\max\left\{n^{\frac{7}{2}},m^{\frac{7}{2}}\right\}\right)$ for SCGD and $\eta=\mathcal{O}\left(T^{-\frac{4}{5}}\right)$, $\beta=\mathcal{O}\left(T^{-\frac{4}{5}}\right), c>4$, $T=\mathcal{O}\left(\max\left\{n^{\frac{5}{2}},m^{\frac{5}{2}}\right\}\right)$ for SCSC to yield the bounds $\mathcal{O}\left(\max\left\{n^{-1},m^{-1}\right\}\cdot\max\left\{n^{\frac{1}{2}},m^{\frac{1}{2}}\right\}\right)$ which is slight larger than $\mathcal{O}\left(n^{-\frac{1}{2}}+m^{-\frac{1}{2}}\right)$. Compared with [21], Theorem 2 enjoys not only tighter bounds but also some more practical parameter selections of $\eta,\beta,T$. There are some experiments [2, 5, 34, 35] to validate this statement. **(1) For** $T$**:** [21] provided some generalization bounds for convex SCGD and SCSC with some impractical $T$ such as $T=O(\max(n^{7/2},m^{7/2}))$ in Theorem 4. While our convex result (Theorem 2) can achieve similar rates even taking $T=O(\max(n,m))$ which better matches some empirical observations (Fig. 1, 2 in [5] and Fig. 2 in [2]). **(2) For** $\eta_t$**:** Theorem 4 in [21] took $\eta_t=T^{-6/7}$ which is too small when $T$ is large. While Theorem 2 takes $\eta_t=O(t^{-1})$ closer to some empirical selections ($\eta_t=O(t^{-3/4})$ in [2, 5] and $\eta_t=O(t^{-1})$ in [34, 35]). **(3) For** $\beta_t$**:** Theorem 4 in [21] took $\beta_t=T^{-4/7}$ which is also too small since [2, 5, 35] empirically select $\beta_t=t^{-1/2}$ or $\beta_t=t^{-1}$. In contrast, Theorem 2 have no special restriction on $\beta_t$.*

*Moreover, the bounds of Theorem 2 are similar to some popular stability bounds in the non-compositional literature. For example, the most classical work [26] achieved the uniform stability bound $\mathcal{O}\left(n^{-1}\log T\right)$ for convex SGD with $\eta_t\leq\mathcal{O}\left(t^{-1}\right)$. [31] showed the uniform stability bound $\mathcal{O}\left(n^{-1}T^{\frac{1}{2}}+n^{-\frac{1}{2}}\sqrt{\log(1/\delta)}\right)$ for convex pairwise SGD with $\eta_t=\mathcal{O}\left(T^{-\frac{1}{2}}\right)$. The proof of Theorem 2 is provided in* Appendix C.

To further weaken our assumptions, the generalization analysis of the convex SCO algorithm is developed into the non-convex setting by introducing the almost co-coercivity property of smooth function.

**Theorem 3.** *Let Assumptions 1, 3 hold. Assume that the randomized algorithms A (Algorithm 1) for SCO problem brings the model sequences $\{w_t\}_{t=1}^T$ and $\left\{w_t^{i,z}\right\}_{t=1}^T \left(\left\{w_t^{j,\bar{z}}\right\}_{t=1}^T\right)$ on S and $S^{i,z} \left(S^{j,\bar{z}}\right)$ with the step size sequence $\{\eta_t\}_{t=1}^T$. For SCGD with $\eta_t \leq \frac{1}{2\rho t}, \rho = \alpha_g L_f + \beta L_g^2 \alpha_f$ and SCSC with $\eta_t \leq \frac{1}{2\rho t}, \rho = \alpha_g L_f + L_g^2 \alpha_f$, the final output $A(S) = w_T$ is uniformly model $(\epsilon_z, \epsilon_{\bar{z}})$-stable with*

$$\epsilon_z = \frac{L_g L_f (eT)^{\frac{1}{2}} \log(eT)}{\rho n} \text{ and } \epsilon_{\bar{z}} = \frac{L_g L_f (eT)^{\frac{1}{2}} \log(eT)}{\rho m}.$$

**Remark 4.** *Under the non-convex setting, Theorem 3 elucidates a satisfactory stability bound $\mathcal{O}\left(\left(n^{-1} + m^{-1}\right) T^{\frac{1}{2}} \log T\right)$ but $T^{\frac{1}{2}}$-times larger than Theorem 2. When $T = \mathcal{O}(\max\{n, m\})$ and ignoring logarithmic terms, this bound is equivalent to $\mathcal{O}\left(\max\left\{n^{-1}, m^{-1}\right\} \cdot \max\left\{n^{\frac{1}{2}}, m^{\frac{1}{2}}\right\}\right)$. Therefore, under the further weakening condition, i.e., non-convexity, Theorem 3 achieves the stability results similar to the ones of the convex SCGD and SCSC in [21] and some non-compositional, non-convex results [26, 30, 32, 33]. The proof of Theorem 3 is provided in* Appendix C.

## 3.2 Learning Guarantees for Black-box SCO

The aforementioned results lay the groundwork for elucidating the impacts of black-box on the learning guarantees for the non-convex SCGD and SCSC, including additional optimization analysis. Three black-box cases shown in Algorithm 1 are considered in this part. Prior to the analysis, we provide the following assumption required by the optimization analysis.

Table 2: Comparisons among the optimization guarantees for SCO algorithms (Thm.-Theorem; Cor.-Corollary; $L, \alpha, V, M$,C.-Lipschitz continuity, smoothness, bounded variance, bounded function, and convexity assumptions; $d_2 = \mathcal{O}(d)$; $\sqrt{}$-has such a property; $\times$-hasn't such a property).

| Algorithm | Optimization | Assumptions | | | | |
|---|---|---|---|---|---|---|
| | | $L$ | $\alpha$ | $V$ | $M$ | C. |
| SCGD ([2] Thm. 8) | $\mathcal{O}\left(T^{-\frac{1}{4}}\right)$ | $\sqrt{}$ | $\sqrt{}$ | $\sqrt{}$ | $\times$ | $\times$ |
| SCSC ([5] Thm. 1) | $\mathcal{O}\left(T^{-\frac{1}{2}}\right)$ | $\sqrt{}$ | $\sqrt{}$ | $\sqrt{}$ | $\times$ | $\times$ |
| SCGD/SCSC ([21] Thm. 3.7) | $\mathcal{O}\left(T^{-\frac{1}{7}}\right)$ | $\sqrt{}$ | $\sqrt{}$ | $\sqrt{}$ | $\sqrt{}$ | $\sqrt{}$ |
| Outer black-box SCGD/SCSC (Thm. 4) | $\mathcal{O}\left(\mu^2 + \frac{d_2}{b}\right)$ | $\sqrt{}$ | $\sqrt{}$ | $\sqrt{}$ | $\sqrt{}$ | $\times$ |
| Inner black-box SCGD/SCSC (Cor. 1) | $\mathcal{O}\left(\mu^2 + \frac{d_2}{b}\right)$ | $\sqrt{}$ | $\sqrt{}$ | $\sqrt{}$ | $\sqrt{}$ | $\times$ |
| Full black-box SCGD/SCSC (Cor. 2) | $\mathcal{O}\left(\mu^4 + \frac{d_2^2}{b^2} + \frac{d_2}{b}\right)$ | $\sqrt{}$ | $\sqrt{}$ | $\sqrt{}$ | $\sqrt{}$ | $\times$ |
| VFL-CZOFO (Ours, Cor. 5) | $\mathcal{O}\left(\mu^2 + \frac{d_2}{b}\right)$ | $\sqrt{}$ | $\sqrt{}$ | $\sqrt{}$ | $\sqrt{}$ | $\times$ |

**Assumption 5.** *For any $w \in \mathcal{W}$ and parameter $\gamma > 0$, the empirical risk $F_S(w)$ satisfies $\mathbb{E}\left[\|\nabla F_S(w)\|^2\right] \geq 2\gamma \mathbb{E}\left[F_S(w) - F_S(w(S))\right].$*

In the absence of the convexity assumption, the gradient of empirical risk $\|\nabla F_S(w)\| = 0$ does not guarantee a global optimal parameter. Assumption 5 postulates that all empirical local optimal parameters are, in fact, empirical global optimal parameters, which prepares for the characterization of $\mathbb{E}[F_S(A(S)) - F_S(w(S))]$.

**Theorem 4.** *Let Assumptions 1, 2, 3, 4, 5 hold. Assume that the randomized algorithms A (Algorithm 1) for SCO problem brings the model sequences $\{w_t\}_{t=1}^T$ and $\left\{w_t^{i,z}\right\}_{t=1}^T \left(\left\{w_t^{j,\bar{z}}\right\}_{t=1}^T\right)$ on S and $S^{i,z} \left(S^{j,\bar{z}}\right)$ with the step size sequence $\{\eta_t\}_{t=1}^T$. For the outer black-box SCGD with*

$\eta_t = \frac{1}{p\gamma t}, p \geq \max\left\{\sqrt{\frac{2\alpha}{\gamma}}, \frac{2(\alpha_g M_f + \beta L_g^2 M_f')}{\mu\gamma}\right\}$ and the outer black-box SCSC with $\eta_t = \frac{1}{p\gamma t}, p \geq$
$\max\left\{\sqrt{\frac{2\alpha}{\gamma}}, \frac{2(\alpha_g M_f + L_g^2 M_f')}{\mu\gamma}\right\}$, the final output $A(S) = w_T$ has the learning guarantee

$$\mathbb{E}\left[F(w_T) - F(w^*)\right] \leq \mathcal{O}\left(\left(n^{-1} + m^{-1}\right) T^{\frac{1}{2}} \log T + n^{-\frac{1}{2}} + \mu^2 + b^{-1} d_2\right),$$

where $d_2 = d - (2p+1)\beta + \left(p + \frac{1}{2}\right)^2 \beta^2$ for SCGD and $d_2 = d - 2p - 1 + \left(p + \frac{1}{2}\right)^2$ for SCSC.

The proof of Theorem 4 is provided in *Appendix D*.

**Remark 5.** *Theorem 4 considers the outer balck-box SCGD and SCSC algorithms where only the gradient of the outer function $f$ is unknown. It establishes the excess risk bound $\mathcal{O}\left(\left(n^{-1} + m^{-1}\right) T^{\frac{1}{2}} \log T + n^{-\frac{1}{2}} + \mu^2 + b^{-1} d_2\right)$. [21] derived the excess risk bound $\mathcal{O}\left(\max\left\{n^{-1}, m^{-1}\right\} \cdot \max\left\{n^{\frac{1}{2}}, m^{\frac{1}{2}}\right\}\right)$ for the convex SCGD and SCSC with the parameter selections in Remark 3. Theorem 4 can derive this bound with the milder condition $T = \mathcal{O}\left(\max\{n, m\}\right)$ and $\eta_t = \mathcal{O}\left(t^{-1}\right)$.*

*For the generalization bound, Theorem 4 is consistent with Theorem 3. As for the optimization bound, [2] proved the convergence rate $\mathbb{E}[\|\nabla F(w_T)\|^2] \leq \mathcal{O}\left(T^{-\frac{1}{4}}\right)$ of SCGD for non-convex problems. [5] proved the convergence rate $T^{-1} \sum_{t=0}^{T-1} \mathbb{E}[\|\nabla F(w_t)\|^2] \leq \mathcal{O}\left(T^{-\frac{1}{2}}\right)$ of SCSC for non-convex problems. Compared with [2, 5], Theorem 4 obtains the black-box-related bound $\mathcal{O}\left(\mu^2 + b^{-1} d_2\right)$ with some smaller learning rates required by our analytical framework. This bound is composed of two dependencies on the estimation distance $\mu$ and the number $b$ of estimation directions in Equation (4), which shows the following two key insights into the impacts of black-box on the learning guarantees of SCO algorithms. Firstly, a small $\mu$ indicates a small distance between the two function values $f_{\bar{z}}(v + \mu u)$ and $f_{\bar{z}}(v)$ selected to make gradient estimation $\tilde{\nabla} f_{\bar{z}}(v)$ in the direction of the unit vector $u$. Therefore, a smaller $\mu$, i.e., a closer estimation distance, brings a better gradient estimation, resulting in a better excess risk bound. Secondly, a large $b$ indicates that plenty of unit vectors $u_l, l = 1, ..., b$ with different directions are selected to make gradient estimation $\tilde{\nabla} f_{\bar{z}}(v)$. Then, a larger $b$, i.e., more estimation directions, leads to a better gradient estimation, resulting in a better excess risk bound. In summary, the bound of Theorem 4 verifies the fact that a better gradient estimation brings a tighter learning guarantee for the outer black-box SCGD and SCSC.*

*Except for the black-box-related term, Theorem 4 chooses the learning rates affected by $\mu$, which is also different from Theorem 4. Although $\mu$ can be very small such as $10^{-4}$ [19], its negative impact on $\eta_t$ can be eliminated by $M_f, M_f'$ since the two inequalities $M_f/\mu \leq L_f$ and $M_f'/\mu \leq \alpha_f$ hold.*

The inner black-box and the full black-box SCO algorithms are studied in the following two corollaries, respectively.

**Corollary 1.** *Let the assumptions of Theorem 4 hold. For the inner black-box SCGD with $\eta_t = \frac{1}{p\gamma t}, p \geq \max\left\{\sqrt{\frac{2\alpha}{\gamma}}, \frac{2(\beta L_g \alpha_f M_g + M_g' L_f)}{\mu\gamma}\right\}$ and the inner black-box SCSC with $\eta_t = \frac{1}{p\gamma t}, p \geq \max\left\{\sqrt{\frac{2\alpha}{\gamma}}, \frac{2(L_g \alpha_f M_g + M_g' L_f)}{\mu\gamma}\right\}$, the final output $A(S) = w_T$ has the learning guarantee*

$$\mathbb{E}\left[F(w_T) - F(w^*)\right] \leq \mathcal{O}\left(\left(n^{-1} + m^{-1}\right) T^{\frac{1}{2}} \log T + n^{-\frac{1}{2}} + \mu^2 + b^{-1} d_2\right),$$

*where $d_2 = d - (2p+1)\beta + \left(p + \frac{1}{2}\right)^2 \beta^2$ for SCGD and $d_2 = d - 2p - 1 + \left(p + \frac{1}{2}\right)^2$ for SCSC.*

**Corollary 2.** *Let the assumptions of Theorem 4 hold. For the full black-box SCGD with $\eta_t = \frac{1}{p\gamma t}, p \geq \max\left\{\sqrt{\frac{2\alpha}{\gamma}}, \frac{2(\beta L_g M_f' M_g + M_f M_g')}{\mu^2\gamma}\right\}$ and the full black-box SCSC with $\eta_t = \frac{1}{p\gamma t}, p \geq \max\left\{\sqrt{\frac{2\alpha}{\gamma}}, \frac{2(L_g M_f' M_g + M_f M_g')}{\mu^2\gamma}\right\}$, the final output $A(S) = w_T$ has the learning guarantee*

$$\mathbb{E}\left[F(w_T) - F(w^*)\right] \leq \mathcal{O}\left(\left(n^{-1} + m^{-1}\right) T^{\frac{1}{2}} \log T + n^{-\frac{1}{2}} + \mu^4 + b^{-2} d_2^2 + b^{-1} d_2\right),$$

where $d_2 = d - 2\sqrt{\left(p + \frac{1}{2}\right)\beta} + \left(p + \frac{1}{2}\right)\beta$ for SCGD and $d_2 = d - 2\sqrt{\left(p + \frac{1}{2}\right)} + p + \frac{1}{2}$ for SCSC.

The proofs of Corollaries 1, 2 are provided in *Appendix D*.

**Remark 6.** *Corollary 1 provides the excess risk bound with the same order as Theorem 4 for the inner black-box SCGD and SCSC. The excess risk bound for the full black-box SCGD and SCSC in Corollary 2 presents the different dependencies on $\mu$ and $b$, i.e., $\mu^4 + b^{-2}d_2^2 + b^{-1}d_2$. These dependencies also comply with the two key insights uncovered by Theorem 4 and Corollary 1. Besides, when $\mu^4 b > d_2$ or $\mu^2 b < d_2$ holds, the term is dominated by $\mu^4$ or $b^{-2}d_2^2$ which denotes a stronger dependence on the gradient estimation quality. Hence, Corollary 2 shows that a larger proportion of unknown gradients may lead to a stronger dependence on the gradient estimation quality.*

## 4 Applications

Considering the existing derivative-free cases in VFL, the analysis framework of SCGD and SCSC is herein applied to two VFL algorithms, FOO-based VFL [16] and VFL-CZOFO [19]. As outlined in Algorithm 2 of *Appendix A*, FOO-based VFL algorithm comprises two components, the $K$ local clients with the model parameters $w^k, k \in [K]$ and the central server with the global model $v$. Concerning the data privacy, different clients do not communicate with each other directly, but exchange information indirectly through a server. For this reason, the objective function of the $k$-th client adopts the same compositional structure $f\left(g\left(w^k\right)\right)$ as SCGD and SCSC. To further ensure data privacy without additional protection techniques, VFL-CZOFO is proposed by introducing the idea of ZOO. Different from the general ZOO-based VFL [18], VFL-CZOFO employs a zeroth-order gradient on the output layer of every client, with other parts utilizing the first-order gradient, which preserves the privacy protection of ZOO while significantly enhancing convergence.

Before stating our remaining results, it should be noted that there are a few differences between the setting of FOO-based VFL (VFL-CZOFO) and the one of SCGD (SCSC). First of all, we set $S = \{z_1, ..., z_n\}$ and $S^{i,z} = \{z_1, ..., z_{i-1}, z'_i, z_{i+1}, ..., z_n\}$ according to the learning paradigm of VFL. Therefore, Theorem 1 is simplified as Corollary 3. Secondly, the update of the outer model (global model) for FOO-based VFL (VFL-CZOFO) is not based on the simple moving average in SCGD (SCSC). Thirdly, Assumptions 1, 3, 4, 5 hold for every client in all $K$ clients. Without loss of generality, we only study the learning guarantees of FOO-based VFL and VFL-CZOFO for the $k$-th client.

**Corollary 3.** *Let Assumption 1 hold. Assume the randomized VFL algorithms $A$ is uniformly model $\epsilon_z$-stable, then, $\mathbb{E}[|F(A(S)) - F_S(A(S))|] \leq L_g L_f \epsilon_z$.*

Corollary 3 gives the relationship between uniform model stability and generalization error under the setting of VFL. The proof of Corollary 3 is omitted since it can be proved by Equation (15) in the proof of Theorem 2.3 [21] without the decomposition in Equation (14). The last two results study the theoretical performance of FOO-based VFL and VFL-CZOFO.

**Corollary 4.** *Let Assumptions 1, 3, 4, 5 hold. For the $k$-th client ($k \in [K]$), assume that the randomized FOO-based VFL algorithm (Algorithm 2) brings the model sequences $\left\{w_t^k\right\}_{t=1}^T$ and $\left\{w_t^{i,z,k}\right\}_{t=1}^T$ on $S$ and $S^{i,z}$ with the step size sequence $\{\eta_t\}_{t=1}^T, \eta_t \leq \frac{1}{2\rho t}, \rho = \alpha_g L_f + L_g^2 \alpha_f$. Then, the final output $A(S) = w_T^k$ of the $k$-th client has the generalization guarantee*

$$\mathbb{E}\left[|F(w_T^k) - F_S(w_T^k)|\right] \leq \mathcal{O}\left(n^{-1}T^{\frac{1}{2}}\log T\right).$$

**Corollary 5.** *Let Assumptions 1, 3, 4, 5 hold. For the $k$-th client ($k \in [K]$), assume that the randomized VFL-CZOFO algorithm (Algorithm 2) brings the model sequences $\left\{w_t^k\right\}_{t=1}^T$ and $\left\{w_t^{i,z,k}\right\}_{t=1}^T$ on $S$ and $S^{i,z}$ with the step size sequence $\{\eta_t\}_{t=1}^T, \eta_t = \frac{1}{p\gamma t}, p \geq \max\left\{\sqrt{\frac{2\alpha}{\gamma}}, \frac{2\left(\alpha_g M_f + L_g^2 M_f'\right)}{\mu\gamma}\right\}$. Then, the final output $A(S) = w_T^k$ of the $k$-th client has the generalization guarantee*

$$\mathbb{E}\left[F(w_T^k) - F(w^{k*})\right] \leq \mathcal{O}\left(n^{-1}T^{\frac{1}{2}}\log T + \mu^2 + b^{-1}d_2\right),$$

where $d_2 = d - 2p - 1 + \left(p + \frac{1}{2}\right)^2$.

The proofs of Corollaries 4, 5 are provided in *Appendix E*.

**Remark 7.** *Corollaries 4, 5 both provide the first stability-based generalization bound $\mathcal{O}\left(n^{-1}T^{\frac{1}{2}}\log T\right)$. As for the optimization bound, Corollary 5 gives $\mathcal{O}\left(\mu^2 + b^{-1}d_2\right)$, which originates from the outer black-box setting.*

*Apart from VFL, the generalization guarantee of zeroth-order horizontal federated learning (HFL) was studied with the algorithmic stability tool. [36] established the systematic theoretical assessments of synchronous federated zeroth-order optimization (FedZO) by developing the on-average model stability analysis. Its generalization bounds and optimization bounds all depend on the two gradient estimation-based parameters $\mu$ and $b$. From our perspective, the complicated compositional structure may be a key factor that makes the generalization bound in Corollary 5 unaffected by the impact of the estimated gradient quality. The reason is that the analysis framework of Theorem 2 in [36] requires a decomposition (Equation (6)) which is hardly achieved due to the compositional structure in our analysis.*

## 5 Conclusions

In this paper, we provide a novel, more effective theoretical analysis of two SCO algorithms, SCGD and SCSC, and their black-box variants utilizing the uniform model stability tool. The analysis framework is applied to the two VFL algorithms, FOO-based VFL and VFL-CZOFO. Our results not only offer satisfactory learning guarantees but also theoretically validate the impacts of black-box that a better gradient estimation brings a tighter learning guarantee and a larger proportion of unknown gradients leads to a stronger dependence on the gradient estimation quality. We hope our study can facilitate future theoretical analyses of SCO problems and inspire new practical algorithms.

## Acknowledgments

This work was supported in part by the National Natural Science Foundation of China (Nos. 12071166 and 62376104) and the Fundamental Research Funds for the Central Universities of China (No. 2662023LXPY005).

## References

[1] Saeed Ghadimi, Guanghui Lan, and Hongchao Zhang. Mini-batch stochastic approximation methods for nonconvex stochastic composite optimization. *Mathematical Programming*, 155(1-2):267–305, 2016.

[2] Mengdi Wang, Ethan X. Fang, and Han Liu. Stochastic compositional gradient descent: algorithms for minimizing compositions of expected-value functions. *Mathematical Programming*, 161(1-2):419–449, 2017.

[3] Mengdi Wang, Ji Liu, and Ethan X. Fang. Accelerating stochastic composition optimization. *Journal of Machine Learning Research*, 18:105:1–105:23, 2017.

[4] Zhe Zhang and Guanghui Lan. Optimal algorithms for convex nested stochastic composite optimization, 2020.

[5] Tianyi Chen, Yuejiao Sun, and Wotao Yin. Solving stochastic compositional optimization is nearly as easy as solving stochastic optimization. *IEEE Transactions on Signal Processing*, 69:4937–4948, 2021.

[6] Wei Jiang, Bokun Wang, Yibo Wang, Lijun Zhang, and Tianbao Yang. Optimal algorithms for stochastic multi-level compositional optimization. In *International Conference on Machine Learning (ICML)*, volume 162, pages 10195–10216, 2022.

[7] Bokun Wang and Tianbao Yang. Finite-sum coupled compositional stochastic optimization: Theory and applications. In *International Conference on Machine Learning (ICML)*, volume 162, pages 23292–23317, 2022.

[8] Tianyi Chen, Yuejiao Sun, and Wotao Yin. Closing the gap: Tighter analysis of alternating stochastic gradient methods for bilevel problems. In *Advances in Neural Information Processing Systems (NeurIPS)*, pages 25294–25307, 2021.

[9] Andrzej Ruszczynski and Alexander Shapiro. Optimization of risk measures. *Mathematics of Operations Research*, 31:433–452, 2006.

[10] Qi Qi, Zhishuai Guo, Yi Xu, Rong Jin, and Tianbao Yang. An online method for a class of distributionally robust optimization with non-convex objectives. In *Advances in Neural Information Processing Systems (NeurIPS)*, pages 10067–10080, 2021.

[11] Purushottam Kar, Bharath K. Sriperumbudur, Prateek Jain, and Harish Karnick. On the generalization ability of online learning algorithms for pairwise loss functions. In *International Conference on Machine Learning (ICML)*, volume 28, pages 441–449, 2013.

[12] Yiming Ying, Longyin Wen, and Siwei Lyu. Stochastic online AUC maximization. In *Advances in Neural Information Processing Systems (NIPS)*, pages 451–459, 2016.

[13] Yunwen Lei and Yiming Ying. Stochastic proximal AUC maximization. *Journal of Machine Learning Research*, 22:61:1–61:45, 2021.

[14] Tianbao Yang and Yiming Ying. AUC maximization in the era of big data and AI: A survey. *ACM Computing Surveys*, 55(8):172:1–172:37, 2023.

[15] Chelsea Finn, Pieter Abbeel, and Sergey Levine. Model-agnostic meta-learning for fast adaptation of deep networks. In *International Conference on Machine Learning (ICML)*, volume 70, pages 1126–1135, 2017.

[16] Tianyi Chen, Xiao Jin, Yuejiao Sun, and Wotao Yin. VAFL: a method of vertical asynchronous federated learning, 2020.

[17] Christoph Dann, Gerhard Neumann, and Jan Peters. Policy evaluation with temporal differences: A survey and comparison. *Journal of Machine Learning Research*, 15(1):809–883, 2014.

[18] Qingsong Zhang, Bin Gu, Zhiyuan Dang, Cheng Deng, and Heng Huang. Desirable companion for vertical federated learning: New zeroth-order gradient based algorithm. In *ACM International Conference on Information and Knowledge Management (CIKM)*, pages 2598–2607, 2021.

[19] Ganyu Wang, Bin Gu, Qingsong Zhang, Xiang Li, Boyu Wang, and Charles X. Ling. A unified solution for privacy and communication efficiency in vertical federated learning. In *Advances in Neural Information Processing Systems (NeurIPS)*, 2023.

[20] V. N. Vapnik. Statistical learning theory. *Encyclopedia of the Sciences of Learning*, 41(4):3185–3185, 1998.

[21] Ming Yang, Xiyuan Wei, Tianbao Yang, and Yiming Ying. Stability and generalization of stochastic compositional gradient descent algorithms, 2023.

[22] Huizhuo Yuan, Xiangru Lian, Chris Junchi Li, Ji Liu, and Wenqing Hu. Efficient smooth non-convex stochastic compositional optimization via stochastic recursive gradient descent. In *Advances in Neural Information Processing Systems (NeurIPS)*, pages 6926–6935, 2019.

[23] Adithya M. Devraj and Jianshu Chen. Stochastic variance reduced primal dual algorithms for empirical composition optimization. In *Advances in Neural Information Processing Systems (NeurIPS)*, pages 9878–9888, 2019.

[24] Saeed Ghadimi, Andrzej Ruszczynski, and Mengdi Wang. A single timescale stochastic approximation method for nested stochastic optimization. *SIAM Journal on Optimization*, 30(1):960–979, 2020.

[25] Andrzej Ruszczynski. A stochastic subgradient method for nonsmooth nonconvex multilevel composition optimization. *SIAM Journal on Control and Optimization*, 59(3):2301–2320, 2021.

[26] Moritz Hardt, Ben Recht, and Yoram Singer. Train faster, generalize better: Stability of stochastic gradient descent. In *International Conference on Machine Learning (ICML)*, volume 48, pages 1225–1234, 2016.

[27] Olivier Bousquet and André Elisseeff. Stability and generalization. *Journal of Machine Learning Research*, 2:499–526, 2002.

[28] Arkadi Nemirovski, Anatoli B. Juditsky, Guanghui Lan, and Alexander Shapiro. Robust stochastic approximation approach to stochastic programming. *SIAM Journal on Optimization*, 19(4):1574–1609, 2009.

[29] Léon Bottou. Large-scale machine learning with stochastic gradient descent. In *International Conference on Computational Statistics (COMPSTAT)*, pages 177–186, 2010.

[30] Yi Zhou, Yingbin Liang, and Huishuai Zhang. Understanding generalization error of SGD in nonconvex optimization. *Machine Learning*, 111(1):345–375, 2022.

[31] Yunwen Lei, Antoine Ledent, and Marius Kloft. Sharper generalization bounds for pairwise learning. In *Advances in Neural Information Processing Systems (NeurIPS)*, pages 21236–21246, 2020.

[32] Yunwen Lei, Mingrui Liu, and Yiming Ying. Generalization guarantee of SGD for pairwise learning. In *Advances in Neural Information Processing Systems (NeurIPS)*, pages 21216–21228, 2021.

[33] Yunwen Lei. Stability and generalization of stochastic optimization with nonconvex and nonsmooth problems. In *Conference on Learning Theory (COLT)*, pages 191–227, 2023.

[34] Zhouyuan Huo, Bin Gu, Ji Liu, and Heng Huang. Accelerated method for stochastic composition optimization with nonsmooth regularization. In *Thirty-Second Conference on Artificial Intelligence (AAAI)*, pages 3287–3294, 2018.

[35] Junyu Zhang and Lin Xiao. A stochastic composite gradient method with incremental variance reduction. In *Advances in Neural Information Processing Systems (NeurIPS)*, pages 9075–9085, 2019.

[36] Jun Chen, Hong Chen, Bin Gu, and Hao Deng. Fine-grained theoretical analysis of federated zeroth-order optimization. In *Advances in Neural Information Processing Systems (NeurIPS)*, 2023.

[37] Dominic Richards and Ilja Kuzborskij. Stability & generalisation of gradient descent for shallow neural networks without the neural tangent kernel. In *Advances in Neural Information Processing Systems (NeurIPS)*, pages 8609–8621, 2021.

[38] Yunwen Lei, Rong Jin, and Yiming Ying. Stability and generalization analysis of gradient methods for shallow neural networks. In *Advances in Neural Information Processing Systems (NeurIPS)*, 2022.

[39] John C. Duchi, Michael I. Jordan, Martin J. Wainwright, and Andre Wibisono. Optimal rates for zero-order convex optimization: The power of two function evaluations. *IEEE Transactions on Information Theory*, 61(5):2788–2806, 2015.

## A  Notations

The main notations of this paper are summarized in Table 3.

Table 3: Summary of main notations involved in this paper.

| Notations | Descriptions |
|---|---|
| $S$ | the training dataset defined as $\{z_1, ..., z_n, \bar{z}_1, ..., \bar{z}_m\}$ |
| $S^{i,z}$ | $S^{i,z} = \{z_1, ..., z_{i-1}, z_i', z_{i+1}, ..., z_n, \bar{z}_1, ..., \bar{z}_m\}$ |
| $S^{j,\bar{z}}$ | $S^{j,\bar{z}} = \{z_1, ..., z_n, \bar{z}_1, ..., \bar{z}_{j-1}, \bar{z}_j', \bar{z}_{j+1}, ..., \bar{z}_m\}$ |
| $b$ | the number of random unit vectors |
| $w, \mathcal{W} \in \mathbb{R}^p$ | the inner model parameter and its hypothesis function space, respectively |
| $v \in \mathbb{R}^d$ | the outer model parameter |
| $g, f$ | the inner function and the outer loss function, respectively |
| $F(w), F_S(w)$ | the population risk and empirical risk based on training dataset $S$, respectively |
| $w(S)$ | the optimal model based on the empirical risk, $w(S) = \arg\min_{w \in \mathcal{W}} F_S(w)$ |
| $w^*$ | the optimal model based on the population risk, $w^* = \arg\min_{w \in \mathcal{W}} F(w)$ |
| $A, A(S)$ | the given algorithm and its output model on $S$, respectively |
| $T$ | the total number of iterations for iterative optimization algorithms |
| $\eta_t$ | the step size at the $t$-th update, $t \in [T-1]$ |
| $w_t$ | the model parameter after $t$-th update, $t \in [T], w_T = A(S)$ |
| $L_g, L_f$ | the parameters of Lipschitz continuity on $g(w), f(v)$, respectively |
| $\alpha_g, \alpha_f, \alpha$ | the parameters of smoothness on $g(w), f(v), f(w)$, respectively |
| $V_g$ | the parameter of bounded variance on $g(w)$ |
| $M_g, M_f$ | the parameters of bounded functions $g(w), f(v)$, respectively |
| $\epsilon_z$ | the parameter of stability |
| $\gamma$ | the parameter of PL condition |
| $[\cdot]$ | $[n] := \{1, ..., n\}$ |
| $e$ | the base of the natural logarithm |
| $\|\cdot\|$ | the Euclidean norm |

The pseudo code of FOO-based VFL and VFL-CZOFO is present in Algorithm 2.

## B  Lemmas

**Lemma 1.** *Assume the function $f$ is convex and $\alpha$-smooth. Then, for any $w, w'$, we have*

$$\langle \nabla f(w) - \nabla f(w'), w - w' \rangle \geq \frac{1}{\alpha} \|\nabla f(w) - \nabla f(w')\|^2.$$

**Lemma 2.** *Let $e$ be the base of the natural logarithm. The following inequalities hold:*

*(a) if $m \in (0,1)$, then $\sum_{k=1}^{t} k^{-m} \leq t^{1-m}/(1-m)$;*

*(b) if $m = 1$, then $\sum_{k=1}^{t} k^{-m} \leq \log(et)$;*

**Algorithm 2** FOO-based VFL / VFL-CZOFO

---

**Require:** $v_1, w_1^k$: initial global model and $K$ local models; $\eta_0, \eta_1$: initial learning rates
   **for all** $t = 1, ..., T - 1$ **do**
      **for all** $k \in [K]$ **in parallel do**
         Randomly select a sample $i_t \in [n]$, obtain $g_{z_{i_t}}(w_t^k)$ and $\nabla g_{z_{i_t}}(w_t^k)$
         Send $g_{z_{i_t}}(w_t^k)$ to server
         **FOO-based VFL:** Receive $\nabla f(g_{z_{i_t}}(w_t^k))$
                  Update $w_{t+1} = w_t - \eta_t \nabla g_{z_{i_t}}(w_t^k) \nabla f(g_{z_{i_t}}(w_t^k))$
         **VFL-CZOFO:** Receive $\tilde{\nabla} f(g_{z_{i_t}}(w_t^k))$
                  Update $w_{t+1} = w_t - \eta_t \nabla g_{z_{i_t}}(w_t^k) \tilde{\nabla} f(g_{z_{i_t}}(w_t^k))$
      **end for**
      Server receives $g_{z_{i_t}}(w_t^k)$ from $K$ clients
      **FOO-based VFL:** Obtain and send $\nabla f(g_{z_{i_t}}(w_t^k))$ to the $k$-th client
      **VFL-CZOFO:** Compute $\tilde{\nabla} f(g_{z_{i_t}}(w_t^k))$ and send it to the $k$-th client
      Obtain $\nabla f(v_t)$ and update $v_{t+1} = v_t - \eta_0 \nabla f(v_t)$
   **end for**
**Ensure:** $K$ final client models $w_T^1, ..., w_T^K$

---

*(c) if $m > 1$, then $\sum_{k=1}^{t} k^{-m} \leq \frac{m}{m-1}$;*

*(d) $\sum_{k=1}^{t} \frac{1}{k+k_0} \leq \log(t+1)$, where $k_0 \geq 1$.*

**Lemma 3.** *[37, 38] Consider the gradient-based optimization method $w_{t+1} = w_t - \eta_t \nabla \hat{f}(w_t)$. For two iteration sequences $\{w_t\}_{t \in [T]}$ and $\{w_t'\}_{t \in [T]}$, if the function $\hat{f}(w_t)$ is $\rho$-smooth, $\eta_t \leq 1/(2\rho)$, and the minimum eigenvalue $\lambda_{\min}\left(\nabla^2 \hat{f}(w_t)\right) \geq -\epsilon$, then*

$$\left\langle w_t - w_t', \nabla \hat{f}(w_t) - \nabla \hat{f}(w_t') \right\rangle$$
$$\geq 2\eta_t \left(1 - \frac{\eta_t \rho}{2}\right) \|\nabla \hat{f}(w_t) - \nabla \hat{f}(w_t')\|^2 - \epsilon \|w_t - w_t' - \eta_t \nabla \hat{f}(w_t) + \eta_t \nabla \hat{f}(w_t')\|^2.$$

**Lemma 4.** *If the function $f$ is $\alpha$-smooth, then, for any $w, w'$, we have*

$$f(w) - f(w') \leq \langle w - w', \nabla f(w') \rangle + \frac{1}{2}\alpha \|w - w'\|^2, \tag{6}$$

$$\frac{1}{2\alpha} \|\nabla f(w)\|^2 \leq f(w) - \inf_{w'} f(w') \leq f(w) \tag{7}$$

*and*

$$\frac{1}{2\alpha} \|\nabla F_S(w)\|^2 \leq F_S(w) - \inf_{w'} F_S(w') \leq F_S(w). \tag{8}$$

**Lemma 5.** *[39] Assume a random vector $X \in \mathbb{R}^d$ is $d$-dimensional uniform distribution. For any $k \in \mathbb{N}$, there holds $\mathbb{E}\left[\|X\|^k\right] = d/(d+k)$.*

**Lemma 6.** *[39] Let $u_l \in \mathbb{R}^d, l \in \{1, 2, ..., b\}$ be i.i.d. random vectors satisfying $d$-dimensional uniform distribution. For every random vector $v \in \mathbb{R}^d$ independent of all $u_l$ and $\beta \in (0, 1)$, the following inequality holds*

$$\mathbb{E}\left[\left\|\frac{1}{b}\sum_{l=1}^{b} \langle v, u_l \rangle u_l - \beta v\right\| \bigg| v\right] \leq \sqrt{\frac{d - 2\beta + \beta^2}{b}} \|v\|.$$

Table 4: The main differences among our main results ($\nabla \hat{f}_1(w_t) = \nabla g(w_t) \frac{1}{b} \sum_{l=1}^{b} \frac{u_{t,l}}{\mu} (f(v_{t+1} + \mu u_{t,l}) - f(v_{t+1}))$, $\nabla \hat{f}_2(w_t) = \frac{1}{b} \sum_{l=1}^{b} \frac{u_{t,l}}{\mu} (g(w_t + \mu u_{t,l}) - g(w_t)) \nabla f(v_{t+1})$, $\nabla \hat{f}_3(w_t) = \frac{1}{b^2} \sum_{l=1}^{b} \frac{u_{t,l}}{\mu} (f(v_{t+1} + \mu u_{t,l}) - f(v_{t+1})) \sum_{l=1}^{b} \frac{u_{t,l}}{\mu} (g(w_t + \mu u_{t,l}) - g(w_t)))$.

| Results | Generalization | Optimization |
|---------|----------------|--------------|
| Thm. 2 | Co-coercivity | — |
| Thm. 3 | Almost co-coercivity | — |
| Thm. 4 | $\nabla \hat{f}_1(w_t)$ | Special decompositions of $\tilde{\nabla} f$ |
| Cor. 1 | $\nabla \hat{f}_2(w_t)$ | Special decompositions of $\tilde{\nabla} g$ |
| Cor. 2 | $\nabla \hat{f}_3(w_t)$ | Combination of Thm. 4 and Cor. 1 |

## C  Proofs for General SCO

**Proof of Theorem 2**:

**(a) SCGD:** As Definition 1, we define $S = \{z_1, ..., z_n, \bar{z}_1, ..., \bar{z}_m\}$, $S^{i,z} = \{z_1, ..., z_{i-1}, z_i', z_{i+1}, ..., z_n, \bar{z}_1, ..., \bar{z}_m\}$ and $S^{j,\bar{z}} = \{z_1, ..., z_n, \bar{z}_1, ..., \bar{z}_{j-1}, \bar{z}_j', \bar{z}_{j+1}, ..., \bar{z}_m\}$ for any $i \in [n], j \in [m]$. The two terms $\mathbb{E}_A \left[ \left\| w_T - w_T^{i,z} \right\| \right]$ and $\mathbb{E}_A \left[ \left\| w_T - w_T^{j,\bar{z}} \right\| \right]$ will be estimated as follows.

**1)** $\mathbb{E}_A \left[ \left\| w_T - w_T^{i,z} \right\| \right]$: There are two cases that need to be considered. Firstly, when $i_t \neq i$, there holds

$$\left\| w_{t+1} - w_{t+1}^{i,z} \right\|^2$$
$$= \left\| w_t - w_t^{i,z} - \eta_t \nabla g_{z_{i_t}}(w_t) \nabla f_{\bar{z}_{j_t}}(v_{t+1}) + \eta_t \nabla g_{z_{i_t}}(w_t^{i,z}) \nabla f_{\bar{z}_{j_t}}(v_{t+1}^{i,z}) \right\|^2$$
$$= \left\| w_t - w_t^{i,z} \right\|^2 + \eta_t^2 \left\| \nabla g_{z_{i_t}}(w_t) \nabla f_{\bar{z}_{j_t}}(v_{t+1}) - \nabla g_{z_{i_t}}(w_t^{i,z}) \nabla f_{\bar{z}_{j_t}}(v_{t+1}^{i,z}) \right\|^2$$
$$- 2\eta_t \left\langle w_t - w_t^{i,z}, \nabla g_{z_{i_t}}(w_t) \nabla f_{\bar{z}_{j_t}}(v_{t+1}) - \nabla g_{z_{i_t}}(w_t^{i,z}) \nabla f_{\bar{z}_{j_t}}(v_{t+1}^{i,z}) \right\rangle.$$

Due to two properties of the function $f(g(z))$, i.e., convexity and smoothness, Lemma 1 implies that

$$\left\langle w_t - w_t^{i,z}, \nabla g_{z_{i_t}}(w_t) \nabla f_{\bar{z}_{j_t}}(v_{t+1}) - \nabla g_{z_{i_t}}(w_t^{i,z}) \nabla f_{\bar{z}_{j_t}}(v_{t+1}^{i,z}) \right\rangle$$
$$= \frac{1}{\beta} \left\langle w_t - w_t^{i,z}, \nabla g_{z_{i_t}}(w_t) \nabla f_{\bar{z}_{j_t}}(g_{z_{i_t}}(w_t)) - \nabla g_{z_{i_t}}(w_t^{i,z}) \nabla f_{\bar{z}_{j_t}}(g_{z_{i_t}}(w_t^{i,z})) \right\rangle$$
$$\geq \frac{1}{\alpha \beta} \left\| \nabla g_{z_{i_t}}(w_t) \nabla f_{\bar{z}_{j_t}}(g_{z_{i_t}}(w_t)) - \nabla g_{z_{i_t}}(w_t^{i,z}) \nabla f_{\bar{z}_{j_t}}(g_{z_{i_t}}(w_t^{i,z})) \right\|^2,$$

where $\beta \nabla f_{\bar{z}_{j_t}}(v_{t+1}) = \nabla f_{\bar{z}_{j_t}}(g_{z_{i_t}}(w_t))$ based on the update of SCGD. Then,

$$\left\| w_{t+1} - w_{t+1}^{i,z} \right\|^2$$
$$\leq \left\| w_t - w_t^{i,z} \right\|^2 + \left( \frac{\eta_t^2}{\beta^2} - \frac{2\eta_t}{\alpha \beta} \right) \left\| \nabla g_{z_{i_t}}(w_t) \nabla f_{\bar{z}_{j_t}}(g_{z_{i_t}}(w_t)) - \nabla g_{z_{i_t}}(w_t^{i,z}) \nabla f_{\bar{z}_{j_t}}(g_{z_{i_t}}(w_t^{i,z})) \right\|^2$$
$$\leq \left\| w_t - w_t^{i,z} \right\|^2,$$

where the second inequality is caused by $\eta_t \leq \frac{2\beta}{\alpha t} \leq \frac{2\beta}{\alpha}$. That is $\left\| w_{t+1} - w_{t+1}^{i,z} \right\| \leq \left\| w_t - w_t^{i,z} \right\|$. Secondly, when $i_t = i$, there holds

$$\left\| w_{t+1} - w_{t+1}^{i,z} \right\|$$

$$
= \left\| w_t - w_t^{i,z} - \eta_t \nabla g_{z_{i_t}}(w_t) \nabla f_{\bar{z}_{j_t}}(v_{t+1}) + \eta_t \nabla g_{z'_{i_t}}(w_t^{i,z}) \nabla f_{\bar{z}_{j_t}}(v_{t+1}^{i,z}) \right\|
$$

$$
\leq \left\| w_t - w_t^{i,z} \right\| + \eta_t \left\| \nabla g_{z_{i_t}}(w_t) \nabla f_{\bar{z}_{j_t}}(v_{t+1}) - \nabla g_{z'_{i_t}}(w_t^{i,z}) \nabla f_{\bar{z}_{j_t}}(v_{t+1}^{i,z}) \right\|
$$

$$
\leq \left\| w_t - w_t^{i,z} \right\| + 2\eta_t \left\| \nabla g_{z_{i_t}}(w_t) \right\| \left\| \nabla f_{\bar{z}_{j_t}}(v_{t+1}) \right\|
$$

$$
\leq \left\| w_t - w_t^{i,z} \right\| + 2 L_g L_f \eta_t.
$$

Combining the above two cases, we can get that

$$
\left\| w_{t+1} - w_{t+1}^{i,z} \right\| \leq \left\| w_t - w_t^{i,z} \right\| \mathbb{I}[i_t \neq i] + \left( \left\| w_t - w_t^{i,z} \right\| + 2 L_g L_f \eta_t \right) \mathbb{I}[i_t = i].
$$

Taking expectation over $i_t$,

$$
\mathbb{E}_{i_t} \left[ \left\| w_{t+1} - w_{t+1}^{i,z} \right\| \right]
$$

$$
\leq \left\| w_t - w_t^{i,z} \right\| \mathbb{E}_{i_t} \left[ \mathbb{I}[i_t \neq i] \right] + \left( \left\| w_t - w_t^{i,z} \right\| + 2 L_g L_f \eta_t \right) \mathbb{E}_{i_t} \left[ \mathbb{I}[i_t = i] \right]
$$

$$
\leq \left\| w_t - w_t^{i,z} \right\| + \frac{2 L_g L_f}{n} \eta_t.
$$

Then, taking expectation over $A$ and taking summation from $t = 1$ to $T - 1$ to get that

$$
\mathbb{E}_A \left[ \left\| w_T - w_T^{i,z} \right\| \right] \leq \mathbb{E}_A \left[ \left\| w_{T-1} - w_{T-1}^{i,z} \right\| \right] + \frac{2 L_g L_f}{n} \eta_t
$$

$$
\leq \sum_{t=1}^{T-1} \frac{2 L_g L_f}{n} \eta_t
$$

$$
= \frac{4 L_g L_f \beta}{\alpha n} \sum_{t=1}^{T-1} \frac{1}{t}
$$

$$
\leq \frac{4 L_g L_f \beta \log(eT)}{\alpha n},
$$

where the last inequality is from Lemma 2 (b).

**2)** $\mathbb{E}_A \left[ \left\| w_T - w_T^{j,\bar{z}} \right\| \right]$: Firstly, when $j_t \neq j$, there holds

$$
\left\| w_{t+1} - w_{t+1}^{j,\bar{z}} \right\|^2
$$

$$
= \left\| w_t - w_t^{j,\bar{z}} - \eta_t \nabla g_{z_{i_t}}(w_t) \nabla f_{\bar{z}_{j_t}}(v_{t+1}) + \eta_t \nabla g_{z_{i_t}}(w_t^{j,\bar{z}}) \nabla f_{\bar{z}_{j_t}}(v_{t+1}^{j,\bar{z}}) \right\|^2
$$

$$
= \left\| w_t - w_t^{j,\bar{z}} \right\|^2 + \eta_t^2 \left\| \nabla g_{z_{i_t}}(w_t) \nabla f_{\bar{z}_{j_t}}(v_{t+1}) - \nabla g_{z_{i_t}}(w_t^{j,\bar{z}}) \nabla f_{\bar{z}_{j_t}}(v_{t+1}^{j,\bar{z}}) \right\|^2
$$

$$
- 2\eta_t \left\langle w_t - w_t^{j,\bar{z}}, \nabla g_{z_{i_t}}(w_t) \nabla f_{\bar{z}_{j_t}}(v_{t+1}) - \nabla g_{z_{i_t}}(w_t^{j,\bar{z}}) \nabla f_{\bar{z}_{j_t}}(v_{t+1}^{j,\bar{z}}) \right\rangle.
$$

Due to two properties of the function $f(g(z))$, i.e., convexity and smoothness, Lemma 1 implies that

$$
\left\langle w_t - w_t^{j,\bar{z}}, \nabla g_{z_{i_t}}(w_t) \nabla f_{\bar{z}_{j_t}}(v_{t+1}) - \nabla g_{z_{i_t}}(w_t^{j,\bar{z}}) \nabla f_{\bar{z}_{j_t}}(v_{t+1}^{j,\bar{z}}) \right\rangle
$$

$$
= \frac{1}{\beta} \left\langle w_t - w_t^{j,\bar{z}}, \nabla g_{z_{i_t}}(w_t) \nabla f_{\bar{z}_{j_t}}(g_{z_{i_t}}(w_t)) - \nabla g_{z_{i_t}}(w_t^{j,\bar{z}}) \nabla f_{\bar{z}_{j_t}}(g_{z_{i_t}}(w_t^{j,\bar{z}})) \right\rangle
$$

$$
\geq \frac{1}{\alpha\beta} \left\| \nabla g_{z_{i_t}}(w_t) \nabla f_{\bar{z}_{j_t}}(g_{z_{i_t}}(w_t)) - \nabla g_{z_{i_t}}(w_t^{j,\bar{z}}) \nabla f_{\bar{z}_{j_t}}(g_{z_{i_t}}(w_t^{j,\bar{z}})) \right\|^2,
$$

where $\beta \nabla f_{\bar{z}_{j_t}}(v_{t+1}) = \nabla f_{\bar{z}_{j_t}}(g_{z_{i_t}}(w_t))$ based on the update of SCGD. Then,

$$
\left\| w_{t+1} - w_{t+1}^{j,\bar{z}} \right\|^2
$$

$$
\leq \left\| w_t - w_t^{j,\bar{z}} \right\|^2 + \left( \frac{\eta_t^2}{\beta^2} - \frac{2\eta_t}{\alpha\beta} \right) \left\| \nabla g_{z_{i_t}}(w_t) \nabla f_{\bar{z}_{j_t}}(g_{z_{i_t}}(w_t)) - \nabla g_{z_{i_t}}(w_t^{j,\bar{z}}) \nabla f_{\bar{z}_{j_t}}(g_{z_{i_t}}(w_t^{j,\bar{z}})) \right\|^2
$$

$$\leq \left\| w_t - w_t^{j,\bar{z}} \right\|^2,$$

where the second inequality is caused by $\eta_t \leq \frac{2\beta}{\alpha t} \leq \frac{2\beta}{\alpha}$. That is $\left\| w_{t+1} - w_{t+1}^{j,\bar{z}} \right\| \leq \left\| w_t - w_t^{j,\bar{z}} \right\|$.
Secondly, when $j_t = j$, there holds

$$\left\| w_{t+1} - w_{t+1}^{j,\bar{z}} \right\|$$
$$= \left\| w_t - w_t^{j,\bar{z}} - \eta_t \nabla g_{z_{i_t}}(w_t)\nabla f_{\bar{z}_{j_t}}(v_{t+1}) + \eta_t \nabla g_{z_{i_t}}(w_t^{j,\bar{z}})\nabla f_{\bar{z}'_{j_t}}(v_{t+1}^{j,\bar{z}}) \right\|$$
$$\leq \left\| w_t - w_t^{j,\bar{z}} \right\| + \eta_t \left\| \nabla g_{z_{i_t}}(w_t)\nabla f_{\bar{z}_{j_t}}(v_{t+1}) - \nabla g_{z_{i_t}}(w_t^{j,\bar{z}})\nabla f_{\bar{z}'_{j_t}}(v_{t+1}^{j,\bar{z}}) \right\|$$
$$\leq \left\| w_t - w_t^{j,\bar{z}} \right\| + 2\eta_t \left\| \nabla g_{z_{i_t}}(w_t) \right\| \left\| \nabla f_{\bar{z}_{j_t}}(v_{t+1}) \right\|$$
$$\leq \left\| w_t - w_t^{j,\bar{z}} \right\| + 2L_g L_f \eta_t.$$

Combining the above two cases, we can get that

$$\left\| w_{t+1} - w_{t+1}^{j,\bar{z}} \right\| \leq \left\| w_t - w_t^{j,\bar{z}} \right\| \mathbb{I}[j_t \neq j] + \left( \left\| w_t - w_t^{j,\bar{z}} \right\| + 2L_g L_f \eta_t \right) \mathbb{I}[j_t = j].$$

Taking expectation over $j_t$,

$$\mathbb{E}_{j_t}\left[ \left\| w_{t+1} - w_{t+1}^{j,\bar{z}} \right\| \right]$$
$$\leq \left\| w_t - w_t^{j,\bar{z}} \right\| \mathbb{E}_{j_t}\left[ \mathbb{I}[j_t \neq j] \right] + \left( \left\| w_t - w_t^{j,\bar{z}} \right\| + 2L_g L_f \eta_t \right) \mathbb{E}_{j_t}\left[ \mathbb{I}[j_t = j] \right]$$
$$\leq \left\| w_t - w_t^{j,\bar{z}} \right\| + \frac{2L_g L_f}{m} \eta_t.$$

Then, taking expectation over $A$ and taking summation from $t = 1$ to $T - 1$ to get that

$$\mathbb{E}_A\left[ \left\| w_T - w_T^{j,\bar{z}} \right\| \right] \leq \mathbb{E}_A\left[ \left\| w_{T-1} - w_{T-1}^{j,\bar{z}} \right\| \right] + \frac{2L_g L_f}{m} \eta_t$$
$$\leq \sum_{t=1}^{T-1} \frac{2L_g L_f}{m} \eta_t$$
$$= \frac{4L_g L_f \beta}{\alpha m} \sum_{t=1}^{T-1} \frac{1}{t}$$
$$\leq \frac{4L_g L_f \beta \log(eT)}{\alpha m}.$$

**(b) SCSC:** Similar to the stability proof of SCGD except for the equation $\nabla f_{\bar{z}_{j_t}}(v_{t+1}) = \nabla f_{\bar{z}_{j_t}}(g_{z_{i_t}}(w_t))$ based on the update of SCSC, we have that, for $\eta_t = \frac{\eta_1}{t} \leq \frac{2}{\alpha t} \leq \frac{2}{\alpha}$,

$$\mathbb{E}_A\left[ \left\| w_T - w_T^{i,z} \right\| \right] \leq \frac{4L_g L_f \log(eT)}{\alpha n}$$

and

$$\mathbb{E}_A\left[ \left\| w_T - w_T^{j,\bar{z}} \right\| \right] \leq \frac{4L_g L_f \log(eT)}{\alpha m}.$$

$\square$

**Proof of Theorem 3**:

**SCGD: 1)** $\mathbb{E}_A\left[ \left\| w_T - w_T^{i,z} \right\| \right]$: Firstly, when $i_t \neq i$, there holds

$$\left\| w_{t+1} - w_{t+1}^{i,z} \right\|^2$$
$$= \left\| w_t - w_t^{i,z} - \eta_t \nabla g_{z_{i_t}}(w_t)\nabla f_{\bar{z}_{j_t}}(v_{t+1}) + \eta_t \nabla g_{z_{i_t}}(w_t^{i,z})\nabla f_{\bar{z}_{j_t}}(v_{t+1}^{i,z}) \right\|^2$$

$$= \left\| w_t - w_t^{i,z} \right\|^2 + \eta_t^2 \left\| \nabla g_{z_{i_t}}(w_t) \nabla f_{\bar{z}_{j_t}}(v_{t+1}) - \nabla g_{z_{i_t}}(w_t^{i,z}) \nabla f_{\bar{z}_{j_t}}(v_{t+1}^{i,z}) \right\|^2$$
$$- 2\eta_t \left\langle w_t - w_t^{i,z}, \nabla g_{z_{i_t}}(w_t) \nabla f_{\bar{z}_{j_t}}(v_{t+1}) - \nabla g_{z_{i_t}}(w_t^{i,z}) \nabla f_{\bar{z}_{j_t}}(v_{t+1}^{i,z}) \right\rangle.$$

Without the convexity of the function $f(g(z))$, Lemma 1 can not hold. An almost co-coercivity of gradient operator (Lemma 3) is introduced to build the relationship between the inner product term $\left\langle w_t - w_t^{i,z}, \nabla g_{z_{i_t}}(w_t) \nabla f_{\bar{z}_{j_t}}(v_{t+1}) - \nabla g_{z_{i_t}}(w_t^{i,z}) \nabla f_{\bar{z}_{j_t}}(v_{t+1}^{i,z}) \right\rangle$ and the two squared terms $\left\| w_t - w_t^{i,z} \right\|^2$, $\left\| \nabla g_{z_{i_t}}(w_t) \nabla f_{\bar{z}_{j_t}}(v_{t+1}) - \nabla g_{z_{i_t}}(w_t^{i,z}) \nabla f_{\bar{z}_{j_t}}(v_{t+1}^{i,z}) \right\|^2$. With Assumption 3, the terms $\nabla g(w_t)$ and $\nabla f(v_{t+1})$ are both differentiable. Thus, $\nabla g(w_t) \nabla f(v_{t+1})$ is also differentiable, which means that it is continuous on its domain. As we all know, a continuous function has primitive functions. Then, it is reasonable to assume that there exists a primitive function $\hat{f}(w_t)$ at least whose derivative function $\nabla \hat{f}(w_t) = \nabla g(w_t) \nabla f(v_{t+1})$. For example, $\frac{1}{\beta} f(v_{t+1})$ is one of the primitive functions $\hat{f}(w_t)$. Then

$$\left\| w_{t+1} - w_{t+1}^{i,z} \right\|^2$$
$$= \left\| w_t - w_t^{i,z} \right\|^2 + \eta_t^2 \left\| \nabla \hat{f}_{z_{i_t}, \bar{z}_{j_t}}(w_t) \nabla \hat{f}_{z_{i_t} \bar{z}_{j_t}}(w_t^{i,z}) \right\|^2$$
$$- 2\eta_t \left\langle w_t - w_t^{i,z}, \nabla \hat{f}_{z_{i_t}, \bar{z}_{j_t}}(w_t) - \nabla \hat{f}_{z_{i_t} \bar{z}_{j_t}}(w_t^{i,z}) \right\rangle. \tag{9}$$

Taking derivative of $\nabla \hat{f}_{z_{i_t}, \bar{z}_{j_t}}(w_t)$ over $w_t$, we get that

$$\nabla^2 \hat{f}_{z_{i_t}, \bar{z}_{j_t}}(w_t) = \nabla^2 g_{z_{i_t}}(w_t) \nabla f_{\bar{z}_{j_t}}(v_{t+1}) + \beta \nabla g_{z_{i_t}}(w_t) \nabla^2 f_{\bar{z}_{j_t}}(v_{t+1}) \nabla g_{z_{i_t}}^\top(w_t).$$

Thus,

$$\left\| \nabla^2 \hat{f}_{z_{i_t}, \bar{z}_{j_t}}(w_t) \right\| \leq \alpha_g L_f + \beta L_g^2 \alpha_f.$$

Let $\rho = \alpha_g L_f + \beta L_g^2 \alpha_f$, then $\hat{f}_{z_{i_t}, \bar{z}_{j_t}}(w_t)$ is $\rho$-smooth. Since $\left\| \nabla^2 \hat{f}_{z_{i_t}, \bar{z}_{j_t}}(w_t) \right\|$ equals to the largest singular value of $\nabla^2 \hat{f}_{z_{i_t}, \bar{z}_{j_t}}(w_t)$, we can know that $\lambda_{\min} \left( \nabla^2 \hat{f}_{z_{i_t}, \bar{z}_{j_t}}(w_t) \right) \geq - \left\| \nabla^2 \hat{f}_{z_{i_t}, \bar{z}_{j_t}}(w_t) \right\| \geq -\rho$. According to Lemma 3, we can get that

$$\left\langle w_t - w_t^{i,z}, \nabla \hat{f}_{z_{i_t}, \bar{z}_{j_t}}(w_t) - \nabla \hat{f}_{z_{i_t} \bar{z}_{j_t}}(w_t^{i,z}) \right\rangle$$
$$\geq 2\eta_t \left( 1 - \frac{\rho \eta_t}{2} \right) \left\| \nabla \hat{f}_{z_{i_t}, \bar{z}_{j_t}}(w_t) - \nabla \hat{f}_{z_{i_t} \bar{z}_{j_t}}(w_t^{i,z}) \right\|^2 - \rho \left\| w_t - w_t^{i,z} - \nabla \hat{f}_{z_{i_t}, \bar{z}_{j_t}}(w_t) + \nabla \hat{f}_{z_{i_t} \bar{z}_{j_t}}(w_t^{i,z}) \right\|^2$$
$$= 2\eta_t \left( 1 - \frac{\rho \eta_t}{2} \right) \left\| \nabla \hat{f}_{z_{i_t}, \bar{z}_{j_t}}(w_t) - \nabla \hat{f}_{z_{i_t} \bar{z}_{j_t}}(w_t^{i,z}) \right\|^2 - \rho \left\| w_{t+1} - w_{t+1}^{i,z} \right\|^2.$$

Now, plugging the above inequality back into Equation (9) yields

$$\left\| w_{t+1} - w_{t+1}^{i,z} \right\|^2$$
$$\leq \left\| w_t - w_t^{i,z} \right\|^2 + \left( \eta_t^2 - 4\eta_t^2 \left( 1 - \frac{\rho \eta_t}{2} \right) \right) \left\| \nabla \hat{f}_{z_{i_t}, \bar{z}_{j_t}}(w_t) \nabla \hat{f}_{z_{i_t} \bar{z}_{j_t}}(w_t^{i,z}) \right\|^2 + 2\rho \eta_t \left\| w_{t+1} - w_{t+1}^{i,z} \right\|^2$$
$$\leq \left\| w_t - w_t^{i,z} \right\|^2 + 2\rho \eta_t \left\| w_{t+1} - w_{t+1}^{i,z} \right\|^2,$$

where the second inequality is due to $\eta_t \leq \frac{1}{2\rho t} \leq \frac{3}{2\rho}$. The above inequality implies

$$\left\| w_{t+1} - w_{t+1}^{i,z} \right\| \leq \frac{1}{\sqrt{1 - 2\rho \eta_t}} \left\| w_t - w_t^{i,z} \right\|,$$

Secondly, when $i_t = i$, there holds

$$\left\| w_{t+1} - w_{t+1}^{i,z} \right\|$$

$$= \left\| w_t - w_t^{i,z} - \eta_t \nabla g_{z_{i_t}}(w_t) \nabla f_{\bar{z}_{j_t}}(v_{t+1}) + \eta_t \nabla g_{z'_{i_t}}(w_t^{i,z}) \nabla f_{\bar{z}_{j_t}}(v_{t+1}^{i,z}) \right\|$$

$$\leq \left\| w_t - w_t^{i,z} \right\| + \eta_t \left\| \nabla g_{z_{i_t}}(w_t) \nabla f_{\bar{z}_{j_t}}(v_{t+1}) - \nabla g_{z'_{i_t}}(w_t^{i,z}) \nabla f_{\bar{z}_{j_t}}(v_{t+1}^{i,z}) \right\|$$

$$\leq \left\| w_t - w_t^{i,z} \right\| + 2\eta_t \left\| \nabla g_{z_{i_t}}(w_t) \right\| \left\| \nabla f_{\bar{z}_{j_t}}(v_{t+1}) \right\|$$

$$\leq \left\| w_t - w_t^{i,z} \right\| + 2 L_g L_f \eta_t.$$

Combining the above two cases, we can get that

$$\left\| w_{t+1} - w_{t+1}^{i,z} \right\| \leq \frac{1}{\sqrt{1 - 2\rho\eta_t}} \left\| w_t - w_t^{i,z} \right\| \mathbb{I}[i_t \neq i] + \left( \left\| w_t - w_t^{i,z} \right\| + 2 L_g L_f \eta_t \right) \mathbb{I}[i_t = i].$$

Taking expectation over $i_t$,

$$\mathbb{E}_{i_t} \left[ \left\| w_{t+1} - w_{t+1}^{i,z} \right\| \right]$$

$$\leq \frac{1}{\sqrt{1 - 2\rho\eta_t}} \left\| w_t - w_t^{i,z} \right\| \mathbb{E}_{i_t} \left[ \mathbb{I}[i_t \neq i] \right] + \left( \left\| w_t - w_t^{i,z} \right\| + 2 L_g L_f \eta_t \right) \mathbb{E}_{i_t} \left[ \mathbb{I}[i_t = i] \right]$$

$$\leq \frac{1}{\sqrt{1 - 2\rho\eta_t}} \left\| w_t - w_t^{i,z} \right\| + \frac{2 L_g L_f}{n} \eta_t.$$

Then, taking expectation over $A$ and taking summation from $t = 1$ to $T - 1$ to get that

$$\mathbb{E}_A \left[ \left\| w_T - w_T^{i,z} \right\| \right] \leq \frac{1}{\sqrt{1 - 2\rho\eta_t}} \mathbb{E}_A \left[ \left\| w_{T-1} - w_{T-1}^{i,z} \right\| \right] + \frac{2 L_g L_f}{n} \eta_t$$

$$\leq \sum_{t=1}^{T-1} \left( \prod_{t'=t+1}^{T-1} \frac{1}{\sqrt{1 - 2\rho\eta_{t'}}} \right) \frac{2 L_g L_f}{n} \eta_t$$

$$\leq \sum_{t=1}^{T-1} \left( \prod_{t'=t+1}^{T-1} \sqrt{1 + \frac{1}{t' - 1}} \right) \frac{L_g L_f}{\rho n} \frac{1}{t}$$

$$\leq \sum_{t=1}^{T-1} \left( \prod_{t'=2}^{T-1} \sqrt{1 + \frac{1}{t' - 1}} \right) \frac{L_g L_f}{\rho n} \frac{1}{t}$$

$$\leq \sqrt{\prod_{t'=2}^{T-1} \exp\left\{ \frac{1}{t' - 1} \right\}} \sum_{t=1}^{T-1} \frac{L_g L_f}{\rho n} \frac{1}{t}$$

$$\leq \sqrt{\exp\left\{ \sum_{t'=2}^{T-1} \frac{1}{t' - 1} \right\}} \frac{L_g L_f}{\rho n} \sum_{t=1}^{T-1} \frac{1}{t}$$

$$\leq \frac{L_g L_f (eT)^{\frac{1}{2}} \log(eT)}{\rho n}, \tag{10}$$

where the fourth inequality is from $e^x \geq 1 + x$.

**2)** $\mathbb{E}_A \left[ \left\| w_T - w_T^{j,\bar{z}} \right\| \right]$: Firstly, when $j_t \neq j$, there holds

$$\left\| w_{t+1} - w_{t+1}^{j,\bar{z}} \right\|^2$$

$$= \left\| w_t - w_t^{j,\bar{z}} - \eta_t \nabla g_{z_{i_t}}(w_t) \nabla f_{\bar{z}_{j_t}}(v_{t+1}) + \eta_t \nabla g_{z_{i_t}}(w_t^{j,\bar{z}}) \nabla f_{\bar{z}_{j_t}}(v_{t+1}^{j,\bar{z}}) \right\|^2$$

$$= \left\| w_t - w_t^{j,\bar{z}} \right\|^2 + \eta_t^2 \left\| \nabla g_{z_{i_t}}(w_t) \nabla f_{\bar{z}_{j_t}}(v_{t+1}) - \nabla g_{z_{i_t}}(w_t^{j,\bar{z}}) \nabla f_{\bar{z}_{j_t}}(v_{t+1}^{j,\bar{z}}) \right\|^2$$

$$- 2\eta_t \left\langle w_t - w_t^{j,\bar{z}}, \nabla g_{z_{i_t}}(w_t) \nabla f_{\bar{z}_{j_t}}(v_{t+1}) - \nabla g_{z_{i_t}}(w_t^{j,\bar{z}}) \nabla f_{\bar{z}_{j_t}}(v_{t+1}^{j,\bar{z}}) \right\rangle.$$

We assume that there exists a primitive function $\hat{f}(w_t)$ at least whose derivative function $\nabla \hat{f}(w_t) = \nabla g(w_t) \nabla f(v_{t+1})$. Then

$$\left\| w_{t+1} - w_{t+1}^{j,\bar{z}} \right\|^2$$

$$= \left\| w_t - w_t^{j,\bar{z}} \right\|^2 + \eta_t^2 \left\| \nabla \hat{f}_{z_{i_t}, \bar{z}_{j_t}}(w_t) \nabla \hat{f}_{z_{i_t} \bar{z}_{j_t}}(w_t^{j,\bar{z}}) \right\|^2$$
$$- 2\eta_t \left\langle w_t - w_t^{j,\bar{z}}, \nabla \hat{f}_{z_{i_t}, \bar{z}_{j_t}}(w_t) - \nabla \hat{f}_{z_{i_t} \bar{z}_{j_t}}(w_t^{j,\bar{z}}) \right\rangle. \tag{11}$$

Taking derivative of $\nabla \hat{f}_{z_{i_t}, \bar{z}_{j_t}}(w_t)$ over $w_t$, we get that

$$\nabla^2 \hat{f}_{z_{i_t}, \bar{z}_{j_t}}(w_t) = \nabla^2 g_{z_{i_t}}(w_t) \nabla f_{\bar{z}_{j_t}}(v_{t+1}) + \beta \nabla g_{z_{i_t}}(w_t) \nabla^2 f_{\bar{z}_{j_t}}(v_{t+1}) \nabla g_{z_{i_t}}^\top(w_t).$$

Thus,

$$\left\| \nabla^2 \hat{f}_{z_{i_t}, \bar{z}_{j_t}}(w_t) \right\| \leq \alpha_g L_f + \beta L_g^2 \alpha_f.$$

Let $\rho = \alpha_g L_f + \beta L_g^2 \alpha_f$, then $\hat{f}_{z_{i_t}, \bar{z}_{j_t}}(w_t)$ is $\rho$-smooth. And we can know that $\lambda_{\min}\left( \nabla^2 \hat{f}_{z_{i_t}, \bar{z}_{j_t}}(w_t) \right) \geq - \left\| \nabla^2 \hat{f}_{z_{i_t}, \bar{z}_{j_t}}(w_t) \right\| \geq -\rho$. According to Lemma 3, we can get that

$$\left\langle w_t - w_t^{j,\bar{z}}, \nabla \hat{f}_{z_{i_t}, \bar{z}_{j_t}}(w_t) - \nabla \hat{f}_{z_{i_t} \bar{z}_{j_t}}(w_t^{j,\bar{z}}) \right\rangle$$
$$\geq 2\eta_t \left( 1 - \frac{\rho \eta_t}{2} \right) \left\| \nabla \hat{f}_{z_{i_t}, \bar{z}_{j_t}}(w_t) - \nabla \hat{f}_{z_{i_t} \bar{z}_{j_t}}(w_t^{j,\bar{z}}) \right\|^2 - \rho \left\| w_t - w_t^{j,\bar{z}} - \nabla \hat{f}_{z_{i_t}, \bar{z}_{j_t}}(w_t) + \nabla \hat{f}_{z_{i_t} \bar{z}_{j_t}}(w_t^{j,\bar{z}}) \right\|^2$$
$$= 2\eta_t \left( 1 - \frac{\rho \eta_t}{2} \right) \left\| \nabla \hat{f}_{z_{i_t}, \bar{z}_{j_t}}(w_t) - \nabla \hat{f}_{z_{i_t} \bar{z}_{j_t}}(w_t^{j,\bar{z}}) \right\|^2 - \rho \left\| w_{t+1} - w_{t+1}^{j,\bar{z}} \right\|^2.$$

Now, plugging the above inequality back into Equation (11) yields

$$\left\| w_{t+1} - w_{t+1}^{j,\bar{z}} \right\|^2$$
$$\leq \left\| w_t - w_t^{j,\bar{z}} \right\|^2 + \left( \eta_t^2 - 4\eta_t^2 \left( 1 - \frac{\rho \eta_t}{2} \right) \right) \left\| \nabla \hat{f}_{z_{i_t}, \bar{z}_{j_t}}(w_t) \nabla \hat{f}_{z_{i_t} \bar{z}_{j_t}}(w_t^{j,\bar{z}}) \right\|^2 + 2\rho \eta_t \left\| w_{t+1} - w_{t+1}^{j,\bar{z}} \right\|^2$$
$$\leq \left\| w_t - w_t^{j,\bar{z}} \right\|^2 + 2\rho \eta_t \left\| w_{t+1} - w_{t+1}^{j,\bar{z}} \right\|^2,$$

where the second inequality is due to $\eta_t \leq \frac{1}{2\rho t} \leq \frac{3}{2\rho}$. The above inequality implies

$$\left\| w_{t+1} - w_{t+1}^{j,\bar{z}} \right\| \leq \frac{1}{\sqrt{1 - 2\rho \eta_t}} \left\| w_t - w_t^{j,\bar{z}} \right\|,$$

Secondly, when $j_t = j$, there holds

$$\left\| w_{t+1} - w_{t+1}^{j,\bar{z}} \right\|$$
$$= \left\| w_t - w_t^{j,\bar{z}} - \eta_t \nabla g_{z_{i_t}}(w_t) \nabla f_{\bar{z}_{j_t}}(v_{t+1}) + \eta_t \nabla g_{z_{i_t}}(w_t^{j,\bar{z}}) \nabla f_{\bar{z}'_{j_t}}(v_{t+1}^{j,\bar{z}}) \right\|$$
$$\leq \left\| w_t - w_t^{j,\bar{z}} \right\| + \eta_t \left\| \nabla g_{z_{i_t}}(w_t) \nabla f_{\bar{z}_{j_t}}(v_{t+1}) - \nabla g_{z'_{i_t}}(w_t^{j,\bar{z}}) \nabla f_{\bar{z}'_{j_t}}(v_{t+1}^{j,\bar{z}}) \right\|$$
$$\leq \left\| w_t - w_t^{j,\bar{z}} \right\| + 2\eta_t \left\| \nabla g_{z_{i_t}}(w_t) \right\| \left\| \nabla f_{\bar{z}_{j_t}}(v_{t+1}) \right\|$$
$$\leq \left\| w_t - w_t^{j,\bar{z}} \right\| + 2 L_g L_f \eta_t.$$

Combining the above two cases, we can get that

$$\left\| w_{t+1} - w_{t+1}^{j,\bar{z}} \right\| \leq \frac{1}{\sqrt{1 - 2\rho \eta_t}} \left\| w_t - w_t^{j,\bar{z}} \right\| \mathbb{I}[j_t \neq j] + \left( \left\| w_t - w_t^{j,\bar{z}} \right\| + 2 L_g L_f \eta_t \right) \mathbb{I}[j_t = j].$$

Taking expectation over $j_t$,

$$\mathbb{E}_{i_t} \left[ \left\| w_{t+1} - w_{t+1}^{j,\bar{z}} \right\| \right]$$
$$\leq \frac{1}{\sqrt{1 - 2\rho \eta_t}} \left\| w_t - w_t^{j,\bar{z}} \right\| \mathbb{E}_{j_t} \left[ \mathbb{I}[j_t \neq j] \right] + \left( \left\| w_t - w_t^{j,\bar{z}} \right\| + 2 L_g L_f \eta_t \right) \mathbb{E}_{j_t} \left[ \mathbb{I}[j_t = j] \right]$$
$$\leq \frac{1}{\sqrt{1 - 2\rho \eta_t}} \left\| w_t - w_t^{j,\bar{z}} \right\| + \frac{2 L_g L_f}{m} \eta_t.$$

Then, taking expectation over $A$ and taking summation from $t = 1$ to $T - 1$ to get that

$$
\mathbb{E}_A \left[ \left\| w_T - w_T^{j,\bar{z}} \right\| \right] \leq \frac{1}{\sqrt{1 - 2\rho\eta_t}} \mathbb{E}_A \left[ \left\| w_{T-1} - w_{T-1}^{j,\bar{z}} \right\| \right] + \frac{2 L_g L_f}{m} \eta_t
$$

$$
\leq \sum_{t=1}^{T-1} \left( \prod_{t'=t+1}^{T-1} \frac{1}{\sqrt{1 - 2\rho\eta_{t'}}} \right) \frac{2 L_g L_f}{m} \eta_t
$$

$$
\leq \sum_{t=1}^{T-1} \left( \prod_{t'=t+1}^{T-1} \sqrt{1 + \frac{1}{t'-1}} \right) \frac{L_g L_f}{\rho m} \frac{1}{t}
$$

$$
\leq \sum_{t=1}^{T-1} \left( \prod_{t'=2}^{T-1} \sqrt{1 + \frac{1}{t'-1}} \right) \frac{L_g L_f}{\rho m} \frac{1}{t}
$$

$$
\leq \sqrt{\prod_{t'=2}^{T-1} \exp\left\{ \frac{1}{t'-1} \right\}} \sum_{t=1}^{T-1} \frac{L_g L_f}{\rho m} \frac{1}{t}
$$

$$
\leq \sqrt{\exp\left\{ \sum_{t'=2}^{T-1} \frac{1}{t'-1} \right\}} \frac{L_g L_f}{\rho m} \sum_{t=1}^{T-1} \frac{1}{t}
$$

$$
\leq \frac{L_g L_f (eT)^{\frac{1}{2}} \log(eT)}{\rho m}, \tag{12}
$$

where the fourth inequality is from $e^x \geq 1 + x$.

**SCSC:** Similar to the stability proof of SCGD except for $\nabla^2 \hat{f}_{z_{i_t}, \bar{z}_{j_t}}(w_t) = \nabla^2 g_{z_{i_t}}(w_t) \nabla f_{\bar{z}_{j_t}}(v_{t+1}) + \nabla g_{z_{i_t}}(w_t) \nabla^2 f_{\bar{z}_{j_t}}(v_{t+1}) \nabla g_{z_{i_t}}^\top(w_t)$ based on the update of SCSC, we have that, for $\eta_t \leq \frac{1}{2\rho t} \leq \frac{3}{2\rho}$, $\rho = \alpha_g L_f + L_g^2 \alpha_f$,

$$
\mathbb{E}_A \left[ \left\| w_T - w_T^{i,z} \right\| \right] \leq \frac{L_g L_f (eT)^{\frac{1}{2}} \log(eT)}{\rho n} \tag{13}
$$

and

$$
\mathbb{E}_A \left[ \left\| w_T - w_T^{j,\bar{z}} \right\| \right] \leq \frac{L_g L_f (eT)^{\frac{1}{2}} \log(eT)}{\rho m}. \tag{14}
$$

$\square$

## D   Proofs for Black-box SCO

**Proof of Theorem 4**:

**SCGD: 1)** $\mathbb{E}_A \left[ \left\| w_T - w_T^{i,z} \right\| \right]$: Firstly, when $i_t \neq i$, there holds

$$
\left\| w_{t+1} - w_{t+1}^{i,z} \right\|^2
$$

$$
= \left\| w_t - w_t^{i,z} - \eta_t \nabla g_{z_{i_t}}(w_t) \tilde{\nabla} f_{\bar{z}_{j_t}}(v_{t+1}) + \eta_t \nabla g_{z_{i_t}}(w_t^{i,z}) \tilde{\nabla} f_{\bar{z}_{j_t}}(v_{t+1}^{i,z}) \right\|^2
$$

$$
= \left\| w_t - w_t^{i,z} \right\|^2 + \eta_t^2 \left\| \nabla g_{z_{i_t}}(w_t) \tilde{\nabla} f_{\bar{z}_{j_t}}(v_{t+1}) - \nabla g_{z_{i_t}}(w_t^{i,z}) \tilde{\nabla} f_{\bar{z}_{j_t}}(v_{t+1}^{i,z}) \right\|^2
$$

$$
- 2\eta_t \left\langle w_t - w_t^{i,z}, \nabla g_{z_{i_t}}(w_t) \tilde{\nabla} f_{\bar{z}_{j_t}}(v_{t+1}) - \nabla g_{z_{i_t}}(w_t^{i,z}) \tilde{\nabla} f_{\bar{z}_{j_t}}(v_{t+1}^{i,z}) \right\rangle
$$

$$
= \left\| w_t - w_t^{i,z} \right\|^2 + \eta_t^2 \left\| \nabla g_{z_{i_t}}(w_t) \frac{1}{b} \sum_{l=1}^{b} \frac{u_{t,l}}{\mu} \left( f_{\bar{z}_{j_t}}(v_{t+1} + \mu u_{t,l}) - f_{\bar{z}_{j_t}}(v_{t+1}) \right) - \nabla g_{z_{i_t}}(w_t^{i,z}) \right.
$$

$$
\left. \frac{1}{b} \sum_{l=1}^{b} \frac{u_{t,l}}{\mu} \left( f_{\bar{z}_{j_t}}(v_{t+1}^{i,z} + \mu u_{t,l}) - f_{\bar{z}_{j_t}}(v_{t+1}^{i,z}) \right) \right\|^2
$$

$$- 2\eta_t \left\langle w_t - w_t^{i,z}, \nabla g_{z_{i_t}}(w_t) \frac{1}{b} \sum_{l=1}^{b} \frac{u_{t,l}}{\mu} \left( f_{\bar{z}_{j_t}}(v_{t+1} + \mu u_{t,l}) - f_{\bar{z}_{j_t}}(v_{t+1}) \right) \right.$$

$$\left. - \nabla g_{z_{i_t}}(w_t^{i,z}) \frac{1}{b} \sum_{l=1}^{b} \frac{u_{t,l}}{\mu} \left( f_{\bar{z}_{j_t}}(v_{t+1}^{i,z} + \mu u_{t,l}) - f_{\bar{z}_{j_t}}(v_{t+1}^{i,z}) \right) \right\rangle .$$

With Assumption 3, the terms $\nabla g(w_t)$ and $f(v_{t+1})$ are both differentiable. Thus, $\nabla g(w_t) \frac{1}{b} \sum_{l=1}^{b} \frac{u_{t,l}}{\mu} \left( f(v_{t+1} + \mu u_{t,l}) - f(v_{t+1}) \right)$ is also differentiable. It is reasonable to assume that there exists a primitive function $\hat{f}(w_t)$ at least whose derivative function $\nabla \hat{f}(w_t) = \nabla g(w_t) \frac{1}{b} \sum_{l=1}^{b} \frac{u_{t,l}}{\mu} \left( f(v_{t+1} + \mu u_{t,l}) - f(v_{t+1}) \right)$. Then

$$\left\| w_{t+1} - w_{t+1}^{i,z} \right\|^2$$

$$= \left\| w_t - w_t^{i,z} \right\|^2 + \eta_t^2 \left\| \nabla \hat{f}_{z_{i_t}, \bar{z}_{j_t}}(w_t) \nabla \hat{f}_{z_{i_t} \bar{z}_{j_t}}(w_t^{i,z}) \right\|^2$$

$$- 2\eta_t \left\langle w_t - w_t^{i,z}, \nabla \hat{f}_{z_{i_t}, \bar{z}_{j_t}}(w_t) - \nabla \hat{f}_{z_{i_t} \bar{z}_{j_t}}(w_t^{i,z}) \right\rangle . \tag{15}$$

Taking derivative of $\nabla \hat{f}_{z_{i_t}, \bar{z}_{j_t}}(w_t)$ over $w_t$, we get that

$$\nabla^2 \hat{f}_{z_{i_t}, \bar{z}_{j_t}}(w_t)$$

$$= \nabla^2 g_{z_{i_t}}(w_t) \frac{1}{b} \sum_{l=1}^{b} \frac{u_{t,l}}{\mu} \left( f_{\bar{z}_{j_t}}(v_{t+1} + \mu u_{t,l}) - f_{\bar{z}_{j_t}}(v_{t+1}) \right)$$

$$+ \beta \nabla g_{z_{i_t}}(w_t) \frac{1}{b} \sum_{l=1}^{b} \frac{u_{t,l}}{\mu} \left( \nabla f_{\bar{z}_{j_t}}(v_{t+1} + \mu u_{t,l}) - \nabla f_{\bar{z}_{j_t}}(v_{t+1}) \right) \nabla g_{z_{i_t}}^{\top}(w_t) .$$

Thus,

$$\left\| \nabla^2 \hat{f}_{z_{i_t}, \bar{z}_{j_t}}(w_t) \right\|$$

$$= \left\| \nabla^2 g_{z_{i_t}}(w_t) \frac{1}{b} \sum_{l=1}^{b} \frac{u_{t,l}}{\mu} \left( f_{\bar{z}_{j_t}}(v_{t+1} + \mu u_{t,l}) - f_{\bar{z}_{j_t}}(v_{t+1}) \right) \right.$$

$$\left. + \beta \nabla g_{z_{i_t}}(w_t) \frac{1}{b} \sum_{l=1}^{b} \frac{u_{t,l}}{\mu} \left( \nabla f_{\bar{z}_{j_t}}(v_{t+1} + \mu u_{t,l}) - \nabla f_{\bar{z}_{j_t}}(v_{t+1}) \right) \nabla g_{z_{i_t}}^{\top}(w_t) \right\|$$

$$\leq \left\| \nabla^2 g_{z_{i_t}}(w_t) \frac{1}{b} \sum_{l=1}^{b} \frac{u_{t,l}}{\mu} \left( f_{\bar{z}_{j_t}}(v_{t+1} + \mu u_{t,l}) - f_{\bar{z}_{j_t}}(v_{t+1}) \right) \right\|$$

$$+ \left\| \beta \nabla g_{z_{i_t}}(w_t) \frac{1}{b} \sum_{l=1}^{b} \frac{u_{t,l}}{\mu} \left( \nabla f_{\bar{z}_{j_t}}(v_{t+1} + \mu u_{t,l}) - \nabla f_{\bar{z}_{j_t}}(v_{t+1}) \right) \nabla g_{z_{i_t}}^{\top}(w_t) \right\|$$

$$\leq \frac{1}{\mu} \left( \alpha_g M_f + \beta L_g^2 M_f' \right) .$$

Let $\rho = \frac{1}{\mu} \left( \alpha_g M_f + \beta L_g^2 M_f' \right)$, then $\hat{f}_{z_{i_t}, \bar{z}_{j_t}}(w_t)$ is $\rho$-smooth. And we can know that $\lambda_{\min} \left( \nabla^2 \hat{f}_{z_{i_t}, \bar{z}_{j_t}}(w_t) \right) \geq - \left\| \nabla^2 \hat{f}_{z_{i_t}, \bar{z}_{j_t}}(w_t) \right\| \geq -\rho$. According to Lemma 3, we can get that

$$\left\langle w_t - w_t^{i,z}, \nabla \hat{f}_{z_{i_t}, \bar{z}_{j_t}}(w_t) - \nabla \hat{f}_{z_{i_t} \bar{z}_{j_t}}(w_t^{i,z}) \right\rangle$$

$$\geq 2\eta_t \left( 1 - \frac{\rho \eta_t}{2} \right) \left\| \nabla \hat{f}_{z_{i_t}, \bar{z}_{j_t}}(w_t) - \nabla \hat{f}_{z_{i_t} \bar{z}_{j_t}}(w_t^{i,z}) \right\|^2 - \rho \left\| w_t - w_t^{i,z} - \nabla \hat{f}_{z_{i_t}, \bar{z}_{j_t}}(w_t) + \nabla \hat{f}_{z_{i_t} \bar{z}_{j_t}}(w_t^{i,z}) \right\|^2$$

$$= 2\eta_t \left( 1 - \frac{\rho \eta_t}{2} \right) \left\| \nabla \hat{f}_{z_{i_t}, \bar{z}_{j_t}}(w_t) - \nabla \hat{f}_{z_{i_t} \bar{z}_{j_t}}(w_t^{i,z}) \right\|^2 - \rho \left\| w_{t+1} - w_{t+1}^{i,z} \right\|^2 .$$

Now, plugging the above inequality back into Equation (15) yields

$$
\left\| w_{t+1} - w_{t+1}^{i,z} \right\|^2
$$
$$
\leq \left\| w_t - w_t^{i,z} \right\|^2 + \left( \eta_t^2 - 4\eta_t^2 \left( 1 - \frac{\rho \eta_t}{2} \right) \right) \left\| \nabla \hat{f}_{z_{i_t}, \bar{z}_{j_t}}(w_t) \nabla \hat{f}_{z_{i_t} \bar{z}_{j_t}}(w_t^{i,z}) \right\|^2 + 2\rho \eta_t \left\| w_{t+1} - w_{t+1}^{i,z} \right\|^2
$$
$$
\leq \left\| w_t - w_t^{i,z} \right\|^2 + 2\rho \eta_t \left\| w_{t+1} - w_{t+1}^{i,z} \right\|^2,
$$

where the second inequality is due to $\eta_t \leq \frac{1}{2\rho t} \leq \frac{3}{2\rho}$. The above inequality implies

$$
\left\| w_{t+1} - w_{t+1}^{i,z} \right\| \leq \frac{1}{\sqrt{1 - 2\rho \eta_t}} \left\| w_t - w_t^{i,z} \right\|,
$$

Secondly, when $i_t = i$, there holds

$$
\left\| w_{t+1} - w_{t+1}^{i,z} \right\|
$$
$$
= \left\| w_t - w_t^{i,z} - \eta_t \nabla g_{z_{i_t}}(w_t) \tilde{\nabla} f_{\bar{z}_{j_t}}(v_{t+1}) + \eta_t \nabla g_{z_{i_t}'}(w_t^{i,z}) \tilde{\nabla} f_{\bar{z}_{j_t}}(v_{t+1}^{i,z}) \right\|
$$
$$
\leq \left\| w_t - w_t^{i,z} \right\| + \eta_t \left\| \nabla g_{z_{i_t}}(w_t) \tilde{\nabla} f_{\bar{z}_{j_t}}(v_{t+1}) - \nabla g_{z_{i_t}'}(w_t^{i,z}) \tilde{\nabla} f_{\bar{z}_{j_t}}(v_{t+1}^{i,z}) \right\|
$$
$$
\leq \left\| w_t - w_t^{i,z} \right\| + 2\eta_t \left\| \nabla g_{z_{i_t}}(w_t) \right\| \left\| \tilde{\nabla} f_{\bar{z}_{j_t}}(v_{t+1}) \right\|
$$
$$
\leq \left\| w_t - w_t^{i,z} \right\| + \frac{2 L_g M_f}{\mu} \eta_t.
$$

Combining the above two cases, we can get that

$$
\left\| w_{t+1} - w_{t+1}^{i,z} \right\| \leq \frac{1}{\sqrt{1 - 2\rho \eta_t}} \left\| w_t - w_t^{i,z} \right\| \mathbb{I}[i_t \neq i] + \left( \left\| w_t - w_t^{i,z} \right\| + \frac{2 L_g M_f}{\mu} \eta_t \right) \mathbb{I}[i_t = i].
$$

Taking expectation over $i_t$,

$$
\mathbb{E}_{i_t} \left[ \left\| w_{t+1} - w_{t+1}^{i,z} \right\| \right]
$$
$$
\leq \frac{1}{\sqrt{1 - 2\rho \eta_t}} \left\| w_t - w_t^{i,z} \right\| \mathbb{E}_{i_t} \left[ \mathbb{I}[i_t \neq i] \right] + \left( \left\| w_t - w_t^{i,z} \right\| + \frac{2 L_g M_f}{\mu} \eta_t \right) \mathbb{E}_{i_t} \left[ \mathbb{I}[i_t = i] \right]
$$
$$
\leq \frac{1}{\sqrt{1 - 2\rho \eta_t}} \left\| w_t - w_t^{i,z} \right\| + \frac{2 L_g M_f}{\mu n} \eta_t.
$$

Then, taking expectation over $A$ and taking summation from $t = 1$ to $T - 1$ to get that

$$
\mathbb{E}_A \left[ \left\| w_T - w_T^{i,z} \right\| \right] \leq \frac{1}{\sqrt{1 - 2\rho \eta_t}} \mathbb{E}_A \left[ \left\| w_{T-1} - w_{T-1}^{i,z} \right\| \right] + \frac{2 L_g M_f}{\mu n} \eta_t
$$
$$
\leq \sum_{t=1}^{T-1} \left( \prod_{t'=t+1}^{T-1} \frac{1}{\sqrt{1 - 2\rho \eta_{t'}}} \right) \frac{2 L_g M_f}{\mu n} \eta_t
$$
$$
\leq \sum_{t=1}^{T-1} \left( \prod_{t'=t+1}^{T-1} \sqrt{1 + \frac{1}{t' - 1}} \right) \frac{L_g M_f}{\rho \mu n} \frac{1}{t}
$$
$$
\leq \sum_{t=1}^{T-1} \left( \prod_{t'=2}^{T-1} \sqrt{1 + \frac{1}{t' - 1}} \right) \frac{L_g M_f}{\rho \mu n} \frac{1}{t}
$$
$$
\leq \sqrt{\prod_{t'=2}^{T-1} \exp \left\{ \frac{1}{t' - 1} \right\}} \sum_{t=1}^{T-1} \frac{L_g M_f}{\rho \mu n} \frac{1}{t}
$$
$$
\leq \sqrt{\exp \left\{ \sum_{t'=2}^{T-1} \frac{1}{t' - 1} \right\}} \frac{L_g M_f}{\rho \mu n} \sum_{t=1}^{T-1} \frac{1}{t}
$$

$$\leq \frac{L_g M_f (eT)^{\frac{1}{2}} \log(eT)}{\rho \mu n}, \tag{16}$$

where the fourth inequality is from $e^x \geq 1 + x$.

**2)** $\mathbb{E}_A \left[ \left\| w_T - w_T^{j,\bar{z}} \right\| \right]$: Firstly, when $j_t \neq j$, there holds

$$\left\| w_{t+1} - w_{t+1}^{j,\bar{z}} \right\|^2$$

$$= \left\| w_t - w_t^{j,\bar{z}} - \eta_t \nabla g_{z_{i_t}}(w_t) \tilde{\nabla} f_{\bar{z}_{j_t}}(v_{t+1}) + \eta_t \nabla g_{z_{i_t}}(w_t^{j,\bar{z}}) \tilde{\nabla} f_{\bar{z}_{j_t}}(v_{t+1}^{j,\bar{z}}) \right\|^2$$

$$= \left\| w_t - w_t^{j,\bar{z}} \right\|^2 + \eta_t^2 \left\| \nabla g_{z_{i_t}}(w_t) \tilde{\nabla} f_{\bar{z}_{j_t}}(v_{t+1}) - \nabla g_{z_{i_t}}(w_t^{j,\bar{z}}) \tilde{\nabla} f_{\bar{z}_{j_t}}(v_{t+1}^{j,\bar{z}}) \right\|^2$$

$$- 2\eta_t \left\langle w_t - w_t^{j,\bar{z}}, \nabla g_{z_{i_t}}(w_t) \tilde{\nabla} f_{\bar{z}_{j_t}}(v_{t+1}) - \nabla g_{z_{i_t}}(w_t^{j,\bar{z}}) \tilde{\nabla} f_{\bar{z}_{j_t}}(v_{t+1}^{j,\bar{z}}) \right\rangle$$

$$= \left\| w_t - w_t^{j,\bar{z}} \right\|^2 + \eta_t^2 \left\| \nabla g_{z_{i_t}}(w_t) \frac{1}{b} \sum_{l=1}^{b} \frac{u_{t,l}}{\mu} \left( f_{\bar{z}_{j_t}}(v_{t+1} + \mu u_{t,l}) - f_{\bar{z}_{j_t}}(v_{t+1}) \right) - \nabla g_{z_{i_t}}(w_t^{j,\bar{z}}) \right.$$

$$\left. \frac{1}{b} \sum_{l=1}^{b} \frac{u_{t,l}}{\mu} \left( f_{\bar{z}_{j_t}}(v_{t+1}^{j,\bar{z}} + \mu u_{t,l}) - f_{\bar{z}_{j_t}}(v_{t+1}^{j,\bar{z}}) \right) \right\|^2$$

$$- 2\eta_t \left\langle w_t - w_t^{j,\bar{z}}, \nabla g_{z_{i_t}}(w_t) \frac{1}{b} \sum_{l=1}^{b} \frac{u_{t,l}}{\mu} \left( f_{\bar{z}_{j_t}}(v_{t+1} + \mu u_{t,l}) - f_{\bar{z}_{j_t}}(v_{t+1}) \right) \right.$$

$$\left. - \nabla g_{z_{i_t}}(w_t^{j,\bar{z}}) \frac{1}{b} \sum_{l=1}^{b} \frac{u_{t,l}}{\mu} \left( f_{\bar{z}_{j_t}}(v_{t+1}^{j,\bar{z}} + \mu u_{t,l}) - f_{\bar{z}_{j_t}}(v_{t+1}^{j,\bar{z}}) \right) \right\rangle.$$

With Assumption 3, the terms $\nabla g(w_t)$ and $f(v_{t+1})$ are both differentiable. Thus, $\nabla g(w_t) \frac{1}{b} \sum_{l=1}^{b} \frac{u_{t,l}}{\mu} \left( f(v_{t+1} + \mu u_{t,l}) - f(v_{t+1}) \right)$ is also differentiable. It is reasonable to assume that there exists a primitive function $\hat{f}(w_t)$ at least whose derivative function $\nabla \hat{f}(w_t) = \nabla g(w_t) \frac{1}{b} \sum_{l=1}^{b} \frac{u_{t,l}}{\mu} \left( f(v_{t+1} + \mu u_{t,l}) - f(v_{t+1}) \right)$. Then

$$\left\| w_{t+1} - w_{t+1}^{j,\bar{z}} \right\|^2$$

$$= \left\| w_t - w_t^{j,\bar{z}} \right\|^2 + \eta_t^2 \left\| \nabla \hat{f}_{z_{i_t}, \bar{z}_{j_t}}(w_t) \nabla \hat{f}_{z_{i_t} \bar{z}_{j_t}}(w_t^{j,\bar{z}}) \right\|^2$$

$$- 2\eta_t \left\langle w_t - w_t^{j,\bar{z}}, \nabla \hat{f}_{z_{i_t}, \bar{z}_{j_t}}(w_t) - \nabla \hat{f}_{z_{i_t} \bar{z}_{j_t}}(w_t^{j,\bar{z}}) \right\rangle. \tag{17}$$

Taking derivative of $\nabla \hat{f}_{z_{i_t}, \bar{z}_{j_t}}(w_t)$ over $w_t$, we get that

$$\nabla^2 \hat{f}_{z_{i_t}, \bar{z}_{j_t}}(w_t)$$

$$= \nabla^2 g_{z_{i_t}}(w_t) \frac{1}{b} \sum_{l=1}^{b} \frac{u_{t,l}}{\mu} \left( f_{\bar{z}_{j_t}}(v_{t+1} + \mu u_{t,l}) - f_{\bar{z}_{j_t}}(v_{t+1}) \right)$$

$$+ \beta \nabla g_{z_{i_t}}(w_t) \frac{1}{b} \sum_{l=1}^{b} \frac{u_{t,l}}{\mu} \left( \nabla f_{\bar{z}_{j_t}}(v_{t+1} + \mu u_{t,l}) - \nabla f_{\bar{z}_{j_t}}(v_{t+1}) \right) \nabla g_{z_{i_t}}^{\top}(w_t).$$

Thus,

$$\left\| \nabla^2 \hat{f}_{z_{i_t}, \bar{z}_{j_t}}(w_t) \right\|$$

$$= \left\| \nabla^2 g_{z_{i_t}}(w_t) \frac{1}{b} \sum_{l=1}^{b} \frac{u_{t,l}}{\mu} \left( f_{\bar{z}_{j_t}}(v_{t+1} + \mu u_{t,l}) - f_{\bar{z}_{j_t}}(v_{t+1}) \right) \right.$$

$$\left. + \beta \nabla g_{z_{i_t}}(w_t) \frac{1}{b} \sum_{l=1}^{b} \frac{u_{t,l}}{\mu} \left( \nabla f_{\bar{z}_{j_t}}(v_{t+1} + \mu u_{t,l}) - \nabla f_{\bar{z}_{j_t}}(v_{t+1}) \right) \nabla g_{z_{i_t}}^{\top}(w_t) \right\|$$

$$\leq \left\| \nabla^2 g_{z_{i_t}}(w_t) \frac{1}{b} \sum_{l=1}^{b} \frac{u_{t,l}}{\mu} \left( f_{\bar{z}_{j_t}}(v_{t+1} + \mu u_{t,l}) - f_{\bar{z}_{j_t}}(v_{t+1}) \right) \right\|$$

$$+ \left\| \beta \nabla g_{z_{i_t}}(w_t) \frac{1}{b} \sum_{l=1}^{b} \frac{u_{t,l}}{\mu} \left( \nabla f_{\bar{z}_{j_t}}(v_{t+1} + \mu u_{t,l}) - \nabla f_{\bar{z}_{j_t}}(v_{t+1}) \right) \nabla g_{z_{i_t}}^{\top}(w_t) \right\|$$

$$\leq \frac{1}{\mu} \left( \alpha_g M_f + \beta L_g^2 M_f' \right).$$

Let $\rho = \frac{1}{\mu} \left( \alpha_g M_f + \beta L_g^2 M_f' \right)$, then $\hat{f}_{z_{i_t}, \bar{z}_{j_t}}(w_t)$ is $\rho$-smooth. And we can know that $\lambda_{\min}\left( \nabla^2 \hat{f}_{z_{i_t}, \bar{z}_{j_t}}(w_t) \right) \geq - \left\| \nabla^2 \hat{f}_{z_{i_t}, \bar{z}_{j_t}}(w_t) \right\| \geq -\rho$. According to Lemma 3, we can get that

$$\left\langle w_t - w_t^{j,\bar{z}}, \nabla \hat{f}_{z_{i_t}, \bar{z}_{j_t}}(w_t) - \nabla \hat{f}_{z_{i_t} \bar{z}_{j_t}}(w_t^{j,\bar{z}}) \right\rangle$$

$$\geq 2\eta_t \left( 1 - \frac{\rho \eta_t}{2} \right) \left\| \nabla \hat{f}_{z_{i_t}, \bar{z}_{j_t}}(w_t) - \nabla \hat{f}_{z_{i_t} \bar{z}_{j_t}}(w_t^{j,\bar{z}}) \right\|^2 - \rho \left\| w_t - w_t^{j,\bar{z}} - \nabla \hat{f}_{z_{i_t}, \bar{z}_{j_t}}(w_t) + \nabla \hat{f}_{z_{i_t} \bar{z}_{j_t}}(w_t^{j,\bar{z}}) \right\|^2$$

$$= 2\eta_t \left( 1 - \frac{\rho \eta_t}{2} \right) \left\| \nabla \hat{f}_{z_{i_t}, \bar{z}_{j_t}}(w_t) - \nabla \hat{f}_{z_{i_t} \bar{z}_{j_t}}(w_t^{j,\bar{z}}) \right\|^2 - \rho \left\| w_{t+1} - w_{t+1}^{j,\bar{z}} \right\|^2.$$

Now, plugging the above inequality back into Equation (17) yields

$$\left\| w_{t+1} - w_{t+1}^{j,\bar{z}} \right\|^2$$

$$\leq \left\| w_t - w_t^{j,\bar{z}} \right\|^2 + \left( \eta_t^2 - 4\eta_t^2 \left( 1 - \frac{\rho \eta_t}{2} \right) \right) \left\| \nabla \hat{f}_{z_{i_t}, \bar{z}_{j_t}}(w_t) \nabla \hat{f}_{z_{i_t} \bar{z}_{j_t}}(w_t^{j,\bar{z}}) \right\|^2 + 2\rho \eta_t \left\| w_{t+1} - w_{t+1}^{j,\bar{z}} \right\|^2$$

$$\leq \left\| w_t - w_t^{j,\bar{z}} \right\|^2 + 2\rho \eta_t \left\| w_{t+1} - w_{t+1}^{j,\bar{z}} \right\|^2,$$

where the second inequality is due to $\eta_t \leq \frac{1}{2\rho t} \leq \frac{3}{2\rho}$. The above inequality implies

$$\left\| w_{t+1} - w_{t+1}^{j,\bar{z}} \right\| \leq \frac{1}{\sqrt{1 - 2\rho \eta_t}} \left\| w_t - w_t^{j,\bar{z}} \right\|,$$

Secondly, when $j_t = j$, there holds

$$\left\| w_{t+1} - w_{t+1}^{j,\bar{z}} \right\|$$

$$= \left\| w_t - w_t^{j,\bar{z}} - \eta_t \nabla g_{z_{i_t}}(w_t) \tilde{\nabla} f_{\bar{z}_{j_t}}(v_{t+1}) + \eta_t \nabla g_{z_{i_t}}(w_t^{j,\bar{z}}) \tilde{\nabla} f_{\bar{z}'_{j_t}}(v_{t+1}^{j,\bar{z}}) \right\|$$

$$\leq \left\| w_t - w_t^{j,\bar{z}} \right\| + \eta_t \left\| \nabla g_{z_{i_t}}(w_t) \tilde{\nabla} f_{\bar{z}_{j_t}}(v_{t+1}) - \nabla g_{z_{i_t}}(w_t^{j,\bar{z}}) \tilde{\nabla} f_{\bar{z}'_{j_t}}(v_{t+1}^{j,\bar{z}}) \right\|$$

$$\leq \left\| w_t - w_t^{j,\bar{z}} \right\| + 2\eta_t \left\| \nabla g_{z_{i_t}}(w_t) \right\| \left\| \tilde{\nabla} f_{\bar{z}_{j_t}}(v_{t+1}) \right\|$$

$$\leq \left\| w_t - w_t^{j,\bar{z}} \right\| + \frac{2 L_g M_f}{\mu} \eta_t.$$

Combining the above two cases, we can get that

$$\left\| w_{t+1} - w_{t+1}^{j,\bar{z}} \right\| \leq \frac{1}{\sqrt{1 - 2\rho \eta_t}} \left\| w_t - w_t^{j,\bar{z}} \right\| \mathbb{I}[j_t \neq j] + \left( \left\| w_t - w_t^{j,\bar{z}} \right\| + \frac{2 L_g M_f}{\mu} \eta_t \right) \mathbb{I}[j_t = j].$$

Taking expectation over $i_t$,

$$\mathbb{E}_{i_t} \left[ \left\| w_{t+1} - w_{t+1}^{j,\bar{z}} \right\| \right]$$

$$\leq \frac{1}{\sqrt{1 - 2\rho \eta_t}} \left\| w_t - w_t^{j,\bar{z}} \right\| \mathbb{E}_{i_t} \left[ \mathbb{I}[j_t \neq j] \right] + \left( \left\| w_t - w_t^{j,\bar{z}} \right\| + \frac{2 L_g M_f}{\mu} \eta_t \right) \mathbb{E}_{i_t} \left[ \mathbb{I}[j_t = j] \right]$$

$$\leq \frac{1}{\sqrt{1 - 2\rho \eta_t}} \left\| w_t - w_t^{j,\bar{z}} \right\| + \frac{2 L_g M_f}{\mu m} \eta_t.$$

Then, taking expectation over $A$ and taking summation from $t = 1$ to $T - 1$ to get that

$$
\begin{aligned}
\mathbb{E}_A\left[\left\|w_T - w_T^{j,\bar{z}}\right\|\right] \leq & \frac{1}{\sqrt{1 - 2\rho\eta_t}}\mathbb{E}_A\left[\left\|w_{T-1} - w_{T-1}^{j,\bar{z}}\right\|\right] + \frac{2L_g M_f}{\mu m}\eta_t \\
\leq & \sum_{t=1}^{T-1}\left(\prod_{t'=t+1}^{T-1}\frac{1}{\sqrt{1 - 2\rho\eta_{t'}}}\right)\frac{2L_g M_f}{\mu m}\eta_t \\
\leq & \sum_{t=1}^{T-1}\left(\prod_{t'=t+1}^{T-1}\sqrt{1 + \frac{1}{t'-1}}\right)\frac{L_g M_f}{\rho\mu m}\frac{1}{t} \\
\leq & \sum_{t=1}^{T-1}\left(\prod_{t'=2}^{T-1}\sqrt{1 + \frac{1}{t'-1}}\right)\frac{L_g M_f}{\rho\mu m}\frac{1}{t} \\
\leq & \sqrt{\prod_{t'=2}^{T-1}\exp\left\{\frac{1}{t'-1}\right\}}\sum_{t=1}^{T-1}\frac{L_g M_f}{\rho\mu m}\frac{1}{t} \\
\leq & \sqrt{\exp\left\{\sum_{t'=2}^{T-1}\frac{1}{t'-1}\right\}}\frac{L_g M_f}{\rho\mu m}\sum_{t=1}^{T-1}\frac{1}{t} \\
\leq & \frac{L_g M_f(eT)^{\frac{1}{2}}\log(eT)}{\rho\mu m},
\end{aligned}
\tag{18}
$$

where the fourth inequality is from $e^x \geq 1 + x$.

Next, we will study the optimization bound. According to Equation (6) in Lemma 4, $\nabla f(w_t) = \beta\nabla g(w_t)\nabla f(v_{t+1})$, second-order Taylor expansion, Lemmas 5 and 6, we provide that, for any $p \geq \sqrt{2}$,

$$
\begin{aligned}
& \mathbb{E}\left[F_S(w_{t+1}) - F_S(w_t)\right] \\
\leq & \mathbb{E}\left[\langle w_{t+1} - w_t, \nabla F_S(w_t)\rangle + \frac{1}{2}\alpha\left\|w_{t+1} - w_t\right\|^2\right] \\
= & \mathbb{E}\left[\left\langle -\eta_t\nabla g_{z_{i_t}}(w_t)\tilde{\nabla}f_{\bar{z}_{j_t}}(v_{t+1}), \nabla F_S(w_t)\right\rangle + \frac{1}{2}\alpha\left\|\eta_t\nabla g_{z_{i_t}}(w_t)\tilde{\nabla}f_{\bar{z}_{j_t}}(v_{t+1})\right\|^2\right] \\
= & \mathbb{E}\left[\left\langle -\eta_t\nabla g_{z_{i_t}}(w_t)\left(\tilde{\nabla}f_{\bar{z}_{j_t}}(v_{t+1}) - \left(p + \frac{1}{2}\right)\beta\nabla f_{\bar{z}_{j_t}}(v_{t+1}) + \left(p + \frac{1}{2}\right)\beta\nabla f_{\bar{z}_{j_t}}(v_{t+1})\right),\right.\right. \\
& \left.\left. \nabla F_S(w_t)\right\rangle + \frac{1}{2}\alpha\left\|\eta_t\nabla g_{z_{i_t}}(w_t)\left(\tilde{\nabla}f_{\bar{z}_{j_t}}(v_{t+1}) - \beta\nabla f_{\bar{z}_{j_t}}(v_{t+1}) + \beta\nabla f_{\bar{z}_{j_t}}(v_{t+1})\right)\right\|^2\right] \\
\leq & \mathbb{E}\left[-\eta_t\left\langle\nabla g_{z_{i_t}}(w_t)\left(\tilde{\nabla}f_{\bar{z}_{j_t}}(v_{t+1}) - \left(p + \frac{1}{2}\right)\beta\nabla f_{\bar{z}_{j_t}}(v_{t+1})\right), \nabla F_S(w_t)\right\rangle\right. \\
& - \left(p + \frac{1}{2}\right)\eta_t\|\nabla F_S(w_t)\|^2 + \alpha\eta_t^2\left\|\nabla g_{z_{i_t}}(w_t)\left(\tilde{\nabla}f_{\bar{z}_{j_t}}(v_{t+1}) - \beta\nabla f_{\bar{z}_{j_t}}(v_{t+1})\right)\right\|^2 \\
& \left. + \alpha\eta_t^2\|\nabla F_S(w_t)\|^2\right] \\
\leq & \mathbb{E}\left[\frac{1}{2}\eta_t\left\|\nabla g_{z_{i_t}}(w_t)\left(\tilde{\nabla}f_{\bar{z}_{j_t}}(v_{t+1}) - \left(p + \frac{1}{2}\right)\beta\nabla f_{\bar{z}_{j_t}}(v_{t+1})\right)\right\|^2 + \frac{1}{2}\eta_t\|\nabla F_S(w_t)\|^2\right. \\
& - \left(p + \frac{1}{2}\right)\eta_t\|\nabla F_S(w_t)\|^2 + \alpha\eta_t^2\left\|\nabla g_{z_{i_t}}(w_t)\left(\tilde{\nabla}f_{\bar{z}_{j_t}}(v_{t+1}) - \beta\nabla f_{\bar{z}_{j_t}}(v_{t+1})\right)\right\|^2 \\
& \left. + \alpha\eta_t^2\|\nabla F_S(w_t)\|^2\right] \\
= & \frac{1}{2}\eta_t\mathbb{E}\left[\left\|\nabla g_{z_{i_t}}(w_t)\left(\tilde{\nabla}f_{\bar{z}_{j_t}}(v_{t+1}) - \left(p + \frac{1}{2}\right)\beta\nabla f_{\bar{z}_{j_t}}(v_{t+1})\right)\right\|^2\right]
\end{aligned}
$$

$$+ \alpha\eta_t^2 \mathbb{E}\left[\left\|\nabla g_{z_{i_t}}(w_t)\left(\tilde{\nabla}f_{\bar{z}_{j_t}}(v_{t+1}) - \beta\nabla f_{\bar{z}_{j_t}}(v_{t+1})\right)\right\|^2\right] + \left(\alpha\eta_t^2 - p\eta_t\right)\mathbb{E}\left[\|\nabla F_S(w_t)\|^2\right]$$

$$=\frac{1}{2}\eta_t \mathbb{E}\left[\left\|\nabla g_{z_{i_t}}(w_t)\left(\frac{1}{b}\sum_{l=1}^{b}\frac{u_{t,l}}{\mu}\left(f_{\bar{z}_{j_t}}(v_{t+1} + \mu u_{t,l}) - f_{\bar{z}_{j_t}}(v_{t+1})\right) - \left(p+\frac{1}{2}\right)\beta\nabla f_{\bar{z}_{j_t}}(v_{t+1})\right)\right\|^2\right]$$

$$+ \alpha\eta_t^2 \mathbb{E}\left[\left\|\nabla g_{z_{i_t}}(w_t)\left(\frac{1}{b}\sum_{l=1}^{b}\frac{u_{t,l}}{\mu}\left(f_{\bar{z}_{j_t}}(v_{t+1} + \mu u_{t,l}) - f_{\bar{z}_{j_t}}(v_{t+1})\right) - \beta\nabla f_{\bar{z}_{j_t}}(v_{t+1})\right)\right\|^2\right]$$

$$+ \left(\alpha\eta_t^2 - p\eta_t\right)\mathbb{E}\left[\|\nabla F_S(w_t)\|^2\right]$$

$$=\frac{1}{2}\eta_t \mathbb{E}\left[\left\|\nabla g_{z_{i_t}}(w_t)\left(\frac{1}{b}\sum_{l=1}^{b}\left(\langle\nabla f_{\bar{z}_{j_t}}(v_{t+1}), u_{t,l}\rangle u_{t,l} + \left(\frac{\mu}{2}(u_{t,l})^\top \nabla^2 f_{\bar{z}_{j_t}}(v)|_{v=v_{t+1}^*} u_{t,l}\right)u_{t,l}\right)\right.\right.\right.$$

$$\left.\left.\left. - \left(p+\frac{1}{2}\right)\beta\nabla f_{\bar{z}_{j_t}}(v_{t+1})\right)\right\|^2\right] + \alpha\eta_t^2 \mathbb{E}\left[\left\|\nabla g_{z_{i_t}}(w_t)\left(\frac{1}{b}\sum_{l=1}^{b}\left(\langle\nabla f_{\bar{z}_{j_t}}(v_{t+1}), u_{t,l}\rangle u_{t,l}\right.\right.\right.\right.$$

$$\left.\left.\left.\left. + \left(\frac{\mu}{2}(u_{t,l})^\top \nabla^2 f_{\bar{z}_{j_t}}(v)|_{v=v_{t+1}^*} u_{t,l}\right)u_{t,l}\right) - \beta\nabla f_{\bar{z}_{j_t}}(v_{t+1})\right)\right\|^2\right] + \left(\alpha\eta_t^2 - p\eta_t\right)\mathbb{E}\left[\|\nabla F_S(w_t)\|^2\right]$$

$$\leq \left(2\alpha\eta_t^2 + \eta_t\right)L_g^2 \mathbb{E}\left[\left\|\frac{1}{b}\sum_{l=1}^{b}\left(\frac{\mu}{2}(u_{t,l})^\top \nabla^2 f_{\bar{z}_{j_t}}(v)|_{v=v_{t+1}^*} u_{t,l}\right)u_{t,l}\right\|^2\right]$$

$$+ \eta_t \mathbb{E}\left[\left\|\nabla g_{z_{i_t}}(w_t)\left(\frac{1}{b}\sum_{l=1}^{b}\langle\nabla f_{\bar{z}_{j_t}}(v_{t+1}), u_{t,l}\rangle u_{t,l} - \left(p+\frac{1}{2}\right)\beta\nabla f_{\bar{z}_{j_t}}(v_{t+1})\right)\right\|^2\right]$$

$$+ 2\alpha\eta_t^2 \mathbb{E}\left[\left\|\nabla g_{z_{i_t}}(w_t)\left(\frac{1}{b}\sum_{l=1}^{b}\langle\nabla f_{\bar{z}_{j_t}}(v_{t+1}), u_{t,l}\rangle u_{t,l} - \beta\nabla f_{\bar{z}_{j_t}}(v_{t+1})\right)\right\|^2\right]$$

$$+ \left(\alpha\eta_t^2 - p\eta_t\right)\mathbb{E}\left[\|\nabla F_S(w_t)\|^2\right]$$

$$\leq \left(2\alpha\eta_t^2 + \eta_t\right)L_g^2 \mathbb{E}\left[\left\|\frac{1}{b}\sum_{l=1}^{b}\left(\frac{\mu}{2}(u_{t,l})^\top \nabla^2 f_{\bar{z}_{j_t}}(v)|_{v=v_{t+1}^*} u_{t,l}\right)u_{t,l}\right\|^2\right]$$

$$+ L_g^2 \eta_t \mathbb{E}\left[\left\|\frac{1}{b}\sum_{l=1}^{b}\langle\nabla f_{\bar{z}_{j_t}}(v_{t+1}), u_{t,l}\rangle u_{t,l} - \left(p+\frac{1}{2}\right)\beta\nabla f_{\bar{z}_{j_t}}(v_{t+1})\right\|^2\right]$$

$$+ 2\alpha L_g^2 \eta_t^2 \mathbb{E}\left[\left\|\left(\frac{1}{b}\sum_{l=1}^{b}\langle\nabla f_{\bar{z}_{j_t}}(v_{t+1}), u_{t,l}\rangle u_{t,l} - \beta\nabla f_{\bar{z}_{j_t}}(v_{t+1})\right)\right\|^2\right]$$

$$+ \left(\alpha\eta_t^2 - p\eta_t\right)\mathbb{E}\left[\|\nabla F_S(w_t)\|^2\right]$$

$$\leq \left(2\alpha\eta_t^2 + \eta_t\right)L_g^2 \mathbb{E}\left[\left\|\frac{1}{b}\sum_{l=1}^{b}\left(\frac{\mu}{2}(u_{t,l})^\top \nabla^2 f_{\bar{z}_{j_t}}(v)|_{v=v_{t+1}^*} u_{t,l}\right)u_{t,l}\right\|^2\right]$$

$$+ \left(\frac{d - 2\beta + \beta^2}{b}2\alpha L_g^2\eta_t^2 + \frac{d - (2p+1)\beta + \left(p+\frac{1}{2}\right)^2\beta^2}{b}L_g^2\eta_t\right)\mathbb{E}\left[\|\nabla f_{\bar{z}_{j_t}}(v_{t+1})\|^2\right]$$

$$+ \left(\alpha\eta_t^2 - p\eta_t\right)\mathbb{E}\left[\|\nabla F_S(w_t)\|^2\right]$$

$$\leq \left(\alpha\eta_t^2 - p\eta_t\right)\mathbb{E}\left[\|\nabla F_S(w_t)\|^2\right] + \frac{L_g^2\mu^2\alpha_f^2}{4}(2\alpha\eta_t^2 + \eta_t)\mathbb{E}\left[\|u_{t,1}\|^6\right]$$

$$+ \frac{d - 2\beta + \beta^2}{b}2\alpha L_f^2 L_g^2\eta_t^2 + \frac{d - (2p+1)\beta + \left(p+\frac{1}{2}\right)^2\beta^2}{b}L_f^2 L_g^2\eta_t$$

$$\leq \left(\alpha \eta_t^2 - p\eta_t\right) \mathbb{E}\left[\|\nabla F_S(w_t)\|^2\right] + \frac{dL_g^2 \mu^2 \alpha_f^2}{4(d+6)}\left(2\alpha\eta_t^2 + \eta_t\right)$$

$$+ \frac{d - 2\beta + \beta^2}{b} 2\alpha L_f^2 L_g^2 \eta_t^2 + \frac{d - (2p+1)\beta + \left(p + \frac{1}{2}\right)^2 \beta^2}{b} L_f^2 L_g^2 \eta_t.$$

Let $d_1 = d - 2\beta + \beta^2$ and $d_2 = d - (2p+1)\beta + \left(p + \frac{1}{2}\right)^2 \beta^2$ to get that

$$\mathbb{E}\left[F_S(w_{t+1}) - F_S(w_t)\right]$$

$$\leq \left(\alpha\eta_t^2 - p\eta_t\right)\mathbb{E}\left[\|\nabla F_S(w_t)\|^2\right] + \frac{dL_g^2 \mu^2 \alpha_f^2}{4(d+6)}\left(2\alpha\eta_t^2 + \eta_t\right) + \frac{L_f^2 L_g^2}{b}\left(2\alpha d_1 \eta_t^2 + d_2 \eta_t\right)$$

$$\leq -\frac{1}{2}p\eta_t \mathbb{E}\left[\|\nabla F_S(w_t)\|^2\right] + \frac{dL_g^2 \mu^2 \alpha_f^2}{4(d+6)}\left(2\alpha\eta_t^2 + \eta_t\right) + \frac{L_f^2 L_g^2}{b}\left(2\alpha d_1 \eta_t^2 + d_2 \eta_t\right)$$

$$\leq -p\gamma\eta_t \mathbb{E}\left[F_S(w_t) - F_S(w(S))\right] + \frac{dL_g^2 \mu^2 \alpha_f^2}{4(d+6)}\left(2\alpha\eta_t^2 + \eta_t\right) + \frac{L_f^2 L_g^2}{b}\left(2\alpha d_1 \eta_t^2 + d_2 \eta_t\right),$$

where the second inequality is due to $\eta_t = \frac{1}{p\gamma t} \leq \frac{p}{2\alpha t} \leq \frac{p}{2\alpha}$ when $p \geq \sqrt{\frac{2\alpha}{\gamma}}$, and the last inequality is from Equation 8 in Lemma 4. Then,

$$\mathbb{E}\left[F_S(w_{t+1}) - F_S(w(S))\right]$$

$$\leq (1 - p\gamma\eta_t)\mathbb{E}\left[F_S(w_t) - F_S(w(S))\right] + \frac{dL_g^2 \mu^2 \alpha_f^2}{4(d+6)}\left(2\alpha\eta_t^2 + \eta_t\right) + \frac{L_f^2 L_g^2}{b}\left(2\alpha d_1 \eta_t^2 + d_2 \eta_t\right)$$

$$= \left(1 - \frac{1}{t}\right)\mathbb{E}\left[F_S(w_t) - F_S(w(S))\right] + \frac{dL_g^2 \mu^2 \alpha_f^2}{4(d+6)}\left(\frac{2}{p^2 \alpha t^2} + \frac{1}{p\alpha t}\right) + \frac{L_f^2 L_g^2}{b}\left(\frac{2d_1}{p^2 \alpha t^2} + \frac{d_2}{p\alpha t}\right).$$

We multiply both sides of the above inequality by $t$ to get that

$$t\mathbb{E}\left[F_S(w_{t+1}) - F_S(w(S))\right]$$

$$\leq (t-1)\mathbb{E}\left[F_S(w_t) - F_S(w(S))\right] + \frac{dL_g^2 \mu^2 \alpha_f^2}{4(d+6)}\left(\frac{2}{p^2 \alpha t} + \frac{1}{p\alpha}\right) + \frac{L_f^2 L_g^2}{b}\left(\frac{2d_1}{p^2 \alpha t} + \frac{d_2}{p\alpha}\right).$$

Then

$$(T-1)\mathbb{E}\left[F_S(w_T) - F_S(w(S))\right]$$

$$\leq (T-2)\mathbb{E}\left[F_S(w_{T-1}) - F_S(w(S))\right] + \frac{dL_g^2 \mu^2 \alpha_f^2}{4(d+6)}\left(\frac{2}{p^2 \alpha (T-1)} + \frac{1}{p\alpha}\right) + \frac{L_f^2 L_g^2}{b}\left(\frac{2d_1}{p^2 \alpha (T-1)} + \frac{d_2}{p\alpha}\right)$$

$$\leq \frac{dL_g^2 \mu^2 \alpha_f^2}{4(d+6)}\left(\sum_{t=1}^{T-1}\frac{2}{p^2 \alpha t} + \frac{T-1}{p\alpha}\right) + \frac{L_f^2 L_g^2}{b}\left(\sum_{t=1}^{T-1}\frac{2d_1}{p^2 \alpha t} + \frac{d_2(T-1)}{p\alpha}\right)$$

$$\leq \frac{dL_g^2 \mu^2 \alpha_f^2}{4(d+6)}\left(\frac{2\log(eT)}{p^2 \alpha} + \frac{T-1}{p\alpha}\right) + \frac{L_f^2 L_g^2}{b}\left(\frac{2d_1 \log(eT)}{p^2 \alpha} + \frac{d_2(T-1)}{p\alpha}\right).$$

That is

$$\mathbb{E}\left[F_S(w_T) - F_S(w(S))\right]$$

$$\leq \frac{dL_g^2 \mu^2 \alpha_f^2}{4(d+6)}\left(\frac{2\log(eT)}{p^2 \alpha (T-1)} + \frac{1}{p\alpha}\right) + \frac{L_f^2 L_g^2}{b}\left(\frac{2d_1 \log(eT)}{p^2 \alpha (T-1)} + \frac{d_2}{p\alpha}\right). \tag{19}$$

Combining Theorem 1, Equations (16), (18) and (19), we can get that

$$\mathbb{E}\left[F(w_T) - F(w^*)\right] \leq \mathcal{O}\left(\left(n^{-1} + m^{-1}\right) T^{\frac{1}{2}}\log T + n^{-\frac{1}{2}} + \mu^2 + b^{-1}d_2\right).$$

**SCSC:** Similar to the stability proof of SCGD except for

$$\nabla^2 \hat{f}_{z_{i_t}, \bar{z}_{j_t}}(w_t)$$

$$=\nabla^2 g_{z_{i_t}}(w_t)\frac{1}{b}\sum_{l=1}^{b}\frac{u_{t,l}}{\mu}\left(f_{\bar{z}_{j_t}}(v_{t+1}+\mu u_{t,l})-f_{\bar{z}_{j_t}}(v_{t+1})\right)$$

$$+\nabla g_{z_{i_t}}(w_t)\frac{1}{b}\sum_{l=1}^{b}\frac{u_{t,l}}{\mu}\left(\nabla f_{\bar{z}_{j_t}}(v_{t+1}+\mu u_{t,l})-\nabla f_{\bar{z}_{j_t}}(v_{t+1})\right)\nabla g_{z_{i_t}}^{\top}(w_t).$$

based on the update of SCSC, we have that, for $\eta_t \leq \frac{1}{2\rho t} \leq \frac{3}{2\rho}, \rho = \frac{1}{\mu}\left(\alpha_g M_f + L_g^2 M_f'\right)$,

$$\mathbb{E}_A\left[\left\|w_T - w_T^{i,z}\right\|\right] \leq \frac{L_g M_f (eT)^{\frac{1}{2}}\log(eT)}{\rho\mu n} \tag{20}$$

and

$$\mathbb{E}_A\left[\left\|w_T - w_T^{j,\bar{z}}\right\|\right] \leq \frac{L_g M_f (eT)^{\frac{1}{2}}\log(eT)}{\rho\mu m}. \tag{21}$$

Similar to the optimization proof of SCGD, we also have that

$$\mathbb{E}\left[F_S(w_T) - F_S(w(S))\right]$$
$$\leq \frac{dL_f^2\mu^2\alpha_g^2}{4(d+6)}\left(\frac{2\log(eT)}{p^2\alpha(T-1)}+\frac{1}{p\alpha}\right)+\frac{L_g^2 L_f^2}{b}\left(\frac{2d_1\log(eT)}{p^2\alpha(T-1)}+\frac{d_2}{p\alpha}\right), \tag{22}$$

where $d_1 = d - 1, d_2 = d - (2p+1) + \left(p+\frac{1}{2}\right)^2$. Combining Theorem 1, Equations (20), (21) and (22), we can get that

$$\mathbb{E}\left[F(w_T) - F(w^*)\right] \leq \mathcal{O}\left(\left(n^{-1}+m^{-1}\right)T^{\frac{1}{2}}\log T + n^{-\frac{1}{2}}+\mu^2+b^{-1}d_2\right).$$

$$\square$$

**Proof of Corollary 1**:

**SCGD: 1)** $\mathbb{E}_A\left[\left\|w_T - w_T^{i,z}\right\|\right]$**:** Firstly, when $i_t \neq i$, there holds

$$\left\|w_{t+1} - w_{t+1}^{i,z}\right\|^2$$

$$=\left\|w_t - w_t^{i,z} - \eta_t\tilde{\nabla}g_{z_{i_t}}(w_t)\nabla f_{\bar{z}_{j_t}}(v_{t+1}) + \eta_t\tilde{\nabla}g_{z_{i_t}}(w_t^{i,z})\nabla f_{\bar{z}_{j_t}}(v_{t+1}^{i,z})\right\|^2$$

$$=\left\|w_t - w_t^{i,z}\right\|^2 + \eta_t^2\left\|\tilde{\nabla}g_{z_{i_t}}(w_t)\nabla f_{\bar{z}_{j_t}}(v_{t+1}) - \tilde{\nabla}g_{z_{i_t}}(w_t^{i,z})\nabla f_{\bar{z}_{j_t}}(v_{t+1}^{i,z})\right\|^2$$

$$-2\eta_t\left\langle w_t - w_t^{i,z}, \tilde{\nabla}g_{z_{i_t}}(w_t)\nabla f_{\bar{z}_{j_t}}(v_{t+1}) - \tilde{\nabla}g_{z_{i_t}}(w_t^{i,z})\nabla f_{\bar{z}_{j_t}}(v_{t+1}^{i,z})\right\rangle$$

$$=\left\|w_t - w_t^{i,z}\right\|^2 + \eta_t^2\left\|\frac{1}{b}\sum_{l=1}^{b}\frac{u_{t,l}}{\mu}\left(g_{z_{i_t}}^{\top}(w_t + \mu u_{t,l}) - g_{z_{i_t}}^{\top}(w_t)\right)\nabla f_{\bar{z}_{j_t}}(v_{t+1})\right.$$

$$\left.-\frac{1}{b}\sum_{l=1}^{b}\frac{u_{t,l}}{\mu}\left(g_{z_{i_t}}(w_t^{i,z} + \mu u_{t,l}) - g_{z_{i_t}}(w_t^{i,z})\right)\nabla f_{\bar{z}_{j_t}}(v_{t+1}^{i,z})\right\|^2$$

$$-2\eta_t\left\langle w_t - w_t^{i,z}, \frac{1}{b}\sum_{l=1}^{b}\frac{u_{t,l}}{\mu}\left(g_{z_{i_t}}(w_t + \mu u_{t,l}) - g_{z_{i_t}}(w_t)\right)\nabla f_{\bar{z}_{j_t}}(v_{t+1})\right.$$

$$\left.-\frac{1}{b}\sum_{l=1}^{b}\frac{u_{t,l}}{\mu}\left(g_{z_{i_t}}(w_t^{i,z} + \mu u_{t,l}) - g_{z_{i_t}}(w_t^{i,z})\right)\nabla f_{\bar{z}_{j_t}}(v_{t+1}^{i,z})\right\rangle.$$

With Assumption 3, the terms $\nabla f(v_{t+1})$ and $g(w_t)$ are both differentiable. Thus, $\frac{1}{b}\sum_{l=1}^{b}\frac{u_{t,l}}{\mu}\left(g(w_t + \mu u_{t,l}) - g(w_t)\right)\nabla f(v_{t+1})$ is also differentiable. It is reasonable to assume

that there exists a primitive function $\hat{f}(w_t)$ at least whose derivative function $\nabla \hat{f}(w_t) = \frac{1}{b}\sum_{l=1}^{b}\frac{u_{t,l}}{\mu}\left(g(w_t+\mu u_{t,l})-g(w_t)\right)\nabla f(v_{t+1})$. Then

$$\left\|w_{t+1}-w_{t+1}^{i,z}\right\|^2$$

$$=\left\|w_t-w_t^{i,z}\right\|^2+\eta_t^2\left\|\nabla\hat{f}_{z_{i_t},\bar{z}_{j_t}}(w_t)\nabla\hat{f}_{z_{i_t}\bar{z}_{j_t}}(w_t^{i,z})\right\|^2$$

$$-2\eta_t\left\langle w_t-w_t^{i,z},\nabla\hat{f}_{z_{i_t},\bar{z}_{j_t}}(w_t)-\nabla\hat{f}_{z_{i_t}\bar{z}_{j_t}}(w_t^{i,z})\right\rangle. \tag{23}$$

Taking derivative of $\nabla\hat{f}_{z_{i_t},\bar{z}_{j_t}}(w_t)$ over $w_t$, we get that

$$\nabla^2\hat{f}_{z_{i_t},\bar{z}_{j_t}}(w_t)$$

$$=\frac{1}{b}\sum_{l=1}^{b}\frac{u_{t,l}}{\mu}\left(\nabla g_{z_{i_t}}(w_t+\mu u_{t,l})-\nabla g_{z_{i_t}}(w_t)\right)\nabla f_{\bar{z}_{j_t}}(v_{t+1})$$

$$+\beta\frac{1}{b}\sum_{l=1}^{b}\frac{u_{t,l}}{\mu}\left(g_{z_{i_t}}(w_t+\mu u_{t,l})-g_{z_{i_t}}(w_t)\right)\nabla^2 f_{\bar{z}_{j_t}}(v_{t+1})\nabla g_{z_{i_t}}^{\top}(w_t).$$

Thus,

$$\left\|\nabla^2\hat{f}_{z_{i_t},\bar{z}_{j_t}}(w_t)\right\|$$

$$=\left\|\frac{1}{b}\sum_{l=1}^{b}\frac{u_{t,l}}{\mu}\left(\nabla g_{z_{i_t}}(w_t+\mu u_{t,l})-\nabla g_{z_{i_t}}(w_t)\right)\nabla f_{\bar{z}_{j_t}}(v_{t+1})\right.$$

$$\left.+\beta\frac{1}{b}\sum_{l=1}^{b}\frac{u_{t,l}}{\mu}\left(g_{z_{i_t}}(w_t+\mu u_{t,l})-g_{z_{i_t}}(w_t)\right)\nabla^2 f_{\bar{z}_{j_t}}(v_{t+1})\nabla g_{z_{i_t}}^{\top}(w_t)\right\|$$

$$\leq\left\|\frac{1}{b}\sum_{l=1}^{b}\frac{u_{t,l}}{\mu}\left(\nabla g_{z_{i_t}}(w_t+\mu u_{t,l})-\nabla g_{z_{i_t}}(w_t)\right)\nabla f_{\bar{z}_{j_t}}(v_{t+1})\right\|$$

$$+\left\|\beta\frac{1}{b}\sum_{l=1}^{b}\frac{u_{t,l}}{\mu}\left(g_{z_{i_t}}(w_t+\mu u_{t,l})-g_{z_{i_t}}(w_t)\right)\nabla^2 f_{\bar{z}_{j_t}}(v_{t+1})\nabla g_{z_{i_t}}^{\top}(w_t)\right\|$$

$$\leq\frac{1}{\mu}\left(\beta L_g\alpha_f M_g+M_g'L_f\right).$$

Let $\rho=\frac{1}{\mu}\left(\beta L_g\alpha_f M_g+M_g'L_f\right)$, then $\hat{f}_{z_{i_t},\bar{z}_{j_t}}(w_t)$ is $\rho$-smooth. And we can know that $\lambda_{\min}\left(\nabla^2\hat{f}_{z_{i_t},\bar{z}_{j_t}}(w_t)\right)\geq-\left\|\nabla^2\hat{f}_{z_{i_t},\bar{z}_{j_t}}(w_t)\right\|\geq-\rho$. According to Lemma 3, we can get that

$$\left\langle w_t-w_t^{i,z},\nabla\hat{f}_{z_{i_t},\bar{z}_{j_t}}(w_t)-\nabla\hat{f}_{z_{i_t}\bar{z}_{j_t}}(w_t^{i,z})\right\rangle$$

$$\geq 2\eta_t\left(1-\frac{\rho\eta_t}{2}\right)\left\|\nabla\hat{f}_{z_{i_t},\bar{z}_{j_t}}(w_t)-\nabla\hat{f}_{z_{i_t}\bar{z}_{j_t}}(w_t^{i,z})\right\|^2-\rho\left\|w_t-w_t^{i,z}-\nabla\hat{f}_{z_{i_t},\bar{z}_{j_t}}(w_t)+\nabla\hat{f}_{z_{i_t}\bar{z}_{j_t}}(w_t^{i,z})\right\|^2$$

$$=2\eta_t\left(1-\frac{\rho\eta_t}{2}\right)\left\|\nabla\hat{f}_{z_{i_t},\bar{z}_{j_t}}(w_t)-\nabla\hat{f}_{z_{i_t}\bar{z}_{j_t}}(w_t^{i,z})\right\|^2-\rho\left\|w_{t+1}-w_{t+1}^{i,z}\right\|^2.$$

Now, plugging the above inequality back into Equation (23) yields

$$\left\|w_{t+1}-w_{t+1}^{i,z}\right\|^2$$

$$\leq\left\|w_t-w_t^{i,z}\right\|^2+\left(\eta_t^2-4\eta_t^2\left(1-\frac{\rho\eta_t}{2}\right)\right)\left\|\nabla\hat{f}_{z_{i_t},\bar{z}_{j_t}}(w_t)\nabla\hat{f}_{z_{i_t}\bar{z}_{j_t}}(w_t^{i,z})\right\|^2+2\rho\eta_t\left\|w_{t+1}-w_{t+1}^{i,z}\right\|^2$$

$$\leq\left\|w_t-w_t^{i,z}\right\|^2+2\rho\eta_t\left\|w_{t+1}-w_{t+1}^{i,z}\right\|^2,$$

where the second inequality is due to $\eta_t\leq\frac{1}{2\rho t}\leq\frac{3}{2\rho}$. The above inequality implies

$$\left\|w_{t+1}-w_{t+1}^{i,z}\right\|\leq\frac{1}{\sqrt{1-2\rho\eta_t}}\left\|w_t-w_t^{i,z}\right\|,$$

Secondly, when $i_t = i$, there holds

$$\left\| w_{t+1} - w_{t+1}^{i,z} \right\|$$

$$= \left\| w_t - w_t^{i,z} - \eta_t \tilde{\nabla} g_{z_{i_t}}(w_t) \nabla f_{\bar{z}_{j_t}}(v_{t+1}) + \eta_t \tilde{\nabla} g_{z'_{i_t}}(w_t^{i,z}) \nabla f_{\bar{z}_{j_t}}(v_{t+1}^{i,z}) \right\|$$

$$\leq \left\| w_t - w_t^{i,z} \right\| + \eta_t \left\| \tilde{\nabla} g_{z_{i_t}}(w_t) \nabla f_{\bar{z}_{j_t}}(v_{t+1}) - \tilde{\nabla} g_{z'_{i_t}}(w_t^{i,z}) \nabla f_{\bar{z}_{j_t}}(v_{t+1}^{i,z}) \right\|$$

$$\leq \left\| w_t - w_t^{i,z} \right\| + 2\eta_t \left\| \tilde{\nabla} g_{z_{i_t}}(w_t) \right\| \left\| \nabla f_{\bar{z}_{j_t}}(v_{t+1}) \right\|$$

$$\leq \left\| w_t - w_t^{i,z} \right\| + \frac{2L_f M_g}{\mu} \eta_t.$$

Combining the above two cases, we can get that

$$\left\| w_{t+1} - w_{t+1}^{i,z} \right\| \leq \frac{1}{\sqrt{1 - 2\rho\eta_t}} \left\| w_t - w_t^{i,z} \right\| \mathbb{I}[i_t \neq i] + \left( \left\| w_t - w_t^{i,z} \right\| + \frac{2L_f M_g}{\mu} \eta_t \right) \mathbb{I}[i_t = i].$$

Taking expectation over $i_t$,

$$\mathbb{E}_{i_t} \left[ \left\| w_{t+1} - w_{t+1}^{i,z} \right\| \right]$$

$$\leq \frac{1}{\sqrt{1 - 2\rho\eta_t}} \left\| w_t - w_t^{i,z} \right\| \mathbb{E}_{i_t} \left[ \mathbb{I}[i_t \neq i] \right] + \left( \left\| w_t - w_t^{i,z} \right\| + \frac{2L_f M_g}{\mu} \eta_t \right) \mathbb{E}_{i_t} \left[ \mathbb{I}[i_t = i] \right]$$

$$\leq \frac{1}{\sqrt{1 - 2\rho\eta_t}} \left\| w_t - w_t^{i,z} \right\| + \frac{2L_f M_g}{\mu n} \eta_t.$$

Then, taking expectation over $A$ and taking summation from $t = 1$ to $T - 1$ to get that

$$\mathbb{E}_A \left[ \left\| w_T - w_T^{i,z} \right\| \right] \leq \frac{1}{\sqrt{1 - 2\rho\eta_t}} \mathbb{E}_A \left[ \left\| w_{T-1} - w_{T-1}^{i,z} \right\| \right] + \frac{2L_f M_g}{\mu n} \eta_t$$

$$\leq \sum_{t=1}^{T-1} \left( \prod_{t'=t+1}^{T-1} \frac{1}{\sqrt{1 - 2\rho\eta_{t'}}} \right) \frac{2L_f M_g}{\mu n} \eta_t$$

$$\leq \sum_{t=1}^{T-1} \left( \prod_{t'=t+1}^{T-1} \sqrt{1 + \frac{1}{t'-1}} \right) \frac{L_f M_g}{\rho \mu n} \frac{1}{t}$$

$$\leq \sum_{t=1}^{T-1} \left( \prod_{t'=2}^{T-1} \sqrt{1 + \frac{1}{t'-1}} \right) \frac{L_f M_g}{\rho \mu n} \frac{1}{t}$$

$$\leq \sqrt{\prod_{t'=2}^{T-1} \exp\left\{ \frac{1}{t'-1} \right\}} \sum_{t=1}^{T-1} \frac{L_f M_g}{\rho \mu n} \frac{1}{t}$$

$$\leq \sqrt{\exp\left\{ \sum_{t'=2}^{T-1} \frac{1}{t'-1} \right\}} \frac{L_f M_g}{\rho \mu n} \sum_{t=1}^{T-1} \frac{1}{t}$$

$$\leq \frac{L_f M_g (eT)^{\frac{1}{2}} \log(eT)}{\rho \mu n}, \tag{24}$$

where the fourth inequality is from $e^x \geq 1 + x$.

**2)** $\mathbb{E}_A \left[ \left\| w_T - w_T^{j,\bar{z}} \right\| \right]$: Firstly, when $j_t \neq j$, there holds

$$\left\| w_{t+1} - w_{t+1}^{j,\bar{z}} \right\|^2$$

$$= \left\| w_t - w_t^{j,\bar{z}} - \eta_t \tilde{\nabla} g_{z_{i_t}}(w_t) \nabla f_{\bar{z}_{j_t}}(v_{t+1}) + \eta_t \tilde{\nabla} g_{z_{i_t}}(w_t^{j,\bar{z}}) \nabla f_{\bar{z}_{j_t}}(v_{t+1}^{j,\bar{z}}) \right\|^2$$

$$= \left\| w_t - w_t^{j,\bar{z}} \right\|^2 + \eta_t^2 \left\| \tilde{\nabla} g_{z_{i_t}}(w_t) \nabla f_{\bar{z}_{j_t}}(v_{t+1}) - \tilde{\nabla} g_{z_{i_t}}(w_t^{j,\bar{z}}) \nabla f_{\bar{z}_{j_t}}(v_{t+1}^{j,\bar{z}}) \right\|^2$$

$$- 2\eta_t \left\langle w_t - w_t^{j,\bar{z}}, \tilde{\nabla} g_{z_{i_t}}(w_t) \nabla f_{\bar{z}_{j_t}}(v_{t+1}) - \tilde{\nabla} g_{z_{i_t}}(w_t^{j,\bar{z}}) \nabla f_{\bar{z}_{j_t}}(v_{t+1}^{j,\bar{z}}) \right\rangle$$

$$= \left\| w_t - w_t^{j,\bar{z}} \right\|^2 + \eta_t^2 \left\| \frac{1}{b}\sum_{l=1}^{b} \frac{u_{t,l}}{\mu} \left( g_{z_{i_t}}(w_t + \mu u_{t,l}) - g_{z_{i_t}}(w_t) \right) \nabla f_{\bar{z}_{j_t}}(v_{t+1}) \right.$$

$$\left. - \frac{1}{b}\sum_{l=1}^{b} \frac{u_{t,l}}{\mu} \left( g_{z_{i_t}}(w_t^{j,\bar{z}} + \mu u_{t,l}) - g_{z_{i_t}}(w_t^{j,\bar{z}}) \right) \nabla f_{\bar{z}_{j_t}}(v_{t+1}^{j,\bar{z}}) \right\|^2$$

$$- 2\eta_t \left\langle w_t - w_t^{j,\bar{z}}, \frac{1}{b}\sum_{l=1}^{b} \frac{u_{t,l}}{\mu} \left( g_{z_{i_t}}(w_t + \mu u_{t,l}) - g_{z_{i_t}}(w_t) \right) \nabla f_{\bar{z}_{j_t}}(v_{t+1}) \right.$$

$$\left. - \frac{1}{b}\sum_{l=1}^{b} \frac{u_{t,l}}{\mu} \left( g_{z_{i_t}}(w_t^{j,\bar{z}} + \mu u_{t,l}) - g_{z_{i_t}}(w_t^{j,\bar{z}}) \right) \nabla f_{\bar{z}_{j_t}}(v_{t+1}^{j,\bar{z}}) \right\rangle.$$

With Assumption 3, the terms $\nabla f(v_{t+1})$ and $g(w_t)$ are both differentiable. Thus, $\frac{1}{b}\sum_{l=1}^{b} \frac{u_{t,l}}{\mu} \left( g(w_t + \mu u_{t,l}) - g(w_t) \right) \nabla f(v_{t+1})$ is also differentiable. It is reasonable to assume that there exists a primitive function $\hat{f}(w_t)$ at least whose derivative function $\nabla \hat{f}(w_t) = \frac{1}{b}\sum_{l=1}^{b} \frac{u_{t,l}}{\mu} \left( g(w_t + \mu u_{t,l}) - g(w_t) \right) \nabla f(v_{t+1})$. Then

$$\left\| w_{t+1} - w_{t+1}^{j,\bar{z}} \right\|^2$$

$$= \left\| w_t - w_t^{j,\bar{z}} \right\|^2 + \eta_t^2 \left\| \nabla \hat{f}_{z_{i_t},\bar{z}_{j_t}}(w_t) \nabla \hat{f}_{z_{i_t}\bar{z}_{j_t}}(w_t^{j,\bar{z}}) \right\|^2$$

$$- 2\eta_t \left\langle w_t - w_t^{j,\bar{z}}, \nabla \hat{f}_{z_{i_t},\bar{z}_{j_t}}(w_t) - \nabla \hat{f}_{z_{i_t}\bar{z}_{j_t}}(w_t^{j,\bar{z}}) \right\rangle. \tag{25}$$

Taking derivative of $\nabla \hat{f}_{z_{i_t},\bar{z}_{j_t}}(w_t)$ over $w_t$, we get that

$$\nabla^2 \hat{f}_{z_{i_t},\bar{z}_{j_t}}(w_t)$$

$$= \frac{1}{b}\sum_{l=1}^{b} \frac{u_{t,l}}{\mu} \left( \nabla g_{z_{i_t}}(w_t + \mu u_{t,l}) - \nabla g_{z_{i_t}}(w_t) \right) \nabla f_{\bar{z}_{j_t}}(v_{t+1})$$

$$+ \beta \frac{1}{b}\sum_{l=1}^{b} \frac{u_{t,l}}{\mu} \left( g_{z_{i_t}}(w_t + \mu u_{t,l}) - g_{z_{i_t}}(w_t) \right) \nabla^2 f_{\bar{z}_{j_t}}(v_{t+1}) \nabla g_{z_{i_t}}^\top(w_t).$$

Thus,

$$\left\| \nabla^2 \hat{f}_{z_{i_t},\bar{z}_{j_t}}(w_t) \right\|$$

$$= \left\| \frac{1}{b}\sum_{l=1}^{b} \frac{u_{t,l}}{\mu} \left( \nabla g_{z_{i_t}}(w_t + \mu u_{t,l}) - \nabla g_{z_{i_t}}(w_t) \right) \nabla f_{\bar{z}_{j_t}}(v_{t+1}) \right.$$

$$\left. + \beta \frac{1}{b}\sum_{l=1}^{b} \frac{u_{t,l}}{\mu} \left( g_{z_{i_t}}(w_t + \mu u_{t,l}) - g_{z_{i_t}}(w_t) \right) \nabla^2 f_{\bar{z}_{j_t}}(v_{t+1}) \nabla g_{z_{i_t}}^\top(w_t) \right\|$$

$$\leq \left\| \frac{1}{b}\sum_{l=1}^{b} \frac{u_{t,l}}{\mu} \left( \nabla g_{z_{i_t}}(w_t + \mu u_{t,l}) - \nabla g_{z_{i_t}}(w_t) \right) \nabla f_{\bar{z}_{j_t}}(v_{t+1}) \right\|$$

$$+ \left\| \beta \frac{1}{b}\sum_{l=1}^{b} \frac{u_{t,l}}{\mu} \left( g_{z_{i_t}}(w_t + \mu u_{t,l}) - g_{z_{i_t}}(w_t) \right) \nabla^2 f_{\bar{z}_{j_t}}(v_{t+1}) \nabla g_{z_{i_t}}^\top(w_t) \right\|$$

$$\leq \frac{1}{\mu} \left( \beta L_g \alpha_f M_g + M_g' L_f \right).$$

Let $\rho = \frac{1}{\mu} \left( \beta L_g \alpha_f M_g + M_g' L_f \right)$, then $\hat{f}_{z_{i_t},\bar{z}_{j_t}}(w_t)$ is $\rho$-smooth. And we can know that $\lambda_{\min}\left( \nabla^2 \hat{f}_{z_{i_t},\bar{z}_{j_t}}(w_t) \right) \geq - \left\| \nabla^2 \hat{f}_{z_{i_t},\bar{z}_{j_t}}(w_t) \right\| \geq -\rho$. According to Lemma 3, we can get that

$$\left\langle w_t - w_t^{j,\bar{z}}, \nabla \hat{f}_{z_{i_t},\bar{z}_{j_t}}(w_t) - \nabla \hat{f}_{z_{i_t}\bar{z}_{j_t}}(w_t^{j,\bar{z}}) \right\rangle$$

$$\geq 2\eta_t \left(1 - \frac{\rho\eta_t}{2}\right) \left\|\nabla \hat{f}_{z_{i_t}, \bar{z}_{j_t}}(w_t) - \nabla \hat{f}_{z_{i_t}\bar{z}_{j_t}}(w_t^{j,\bar{z}})\right\|^2 - \rho \left\|w_t - w_t^{j,\bar{z}} - \nabla \hat{f}_{z_{i_t}, \bar{z}_{j_t}}(w_t) + \nabla \hat{f}_{z_{i_t}\bar{z}_{j_t}}(w_t^{j,\bar{z}})\right\|^2$$

$$= 2\eta_t \left(1 - \frac{\rho\eta_t}{2}\right) \left\|\nabla \hat{f}_{z_{i_t}, \bar{z}_{j_t}}(w_t) - \nabla \hat{f}_{z_{i_t}\bar{z}_{j_t}}(w_t^{j,\bar{z}})\right\|^2 - \rho \left\|w_{t+1} - w_{t+1}^{j,\bar{z}}\right\|^2.$$

Now, plugging the above inequality back into Equation (25) yields

$$\left\|w_{t+1} - w_{t+1}^{j,\bar{z}}\right\|^2$$

$$\leq \left\|w_t - w_t^{j,\bar{z}}\right\|^2 + \left(\eta_t^2 - 4\eta_t^2 \left(1 - \frac{\rho\eta_t}{2}\right)\right) \left\|\nabla \hat{f}_{z_{i_t}, \bar{z}_{j_t}}(w_t) \nabla \hat{f}_{z_{i_t}\bar{z}_{j_t}}(w_t^{j,\bar{z}})\right\|^2 + 2\rho\eta_t \left\|w_{t+1} - w_{t+1}^{j,\bar{z}}\right\|^2$$

$$\leq \left\|w_t - w_t^{j,\bar{z}}\right\|^2 + 2\rho\eta_t \left\|w_{t+1} - w_{t+1}^{j,\bar{z}}\right\|^2,$$

where the second inequality is due to $\eta_t \leq \frac{1}{2\rho t} \leq \frac{3}{2\rho}$. The above inequality implies

$$\left\|w_{t+1} - w_{t+1}^{j,\bar{z}}\right\| \leq \frac{1}{\sqrt{1 - 2\rho\eta_t}} \left\|w_t - w_t^{j,\bar{z}}\right\|,$$

Secondly, when $j_t = j$, there holds

$$\left\|w_{t+1} - w_{t+1}^{j,\bar{z}}\right\|$$

$$= \left\|w_t - w_t^{j,\bar{z}} - \eta_t \tilde{\nabla} g_{z_{i_t}}(w_t) \nabla f_{\bar{z}_{j_t}}(v_{t+1}) + \eta_t \tilde{\nabla} g_{z_{i_t}}(w_t^{j,\bar{z}}) \nabla f_{\bar{z}'_{j_t}}(v_{t+1}^{j,\bar{z}})\right\|$$

$$\leq \left\|w_t - w_t^{j,\bar{z}}\right\| + \eta_t \left\|\tilde{\nabla} g_{z_{i_t}}(w_t) \nabla f_{\bar{z}_{j_t}}(v_{t+1}) - \tilde{\nabla} g_{z_{i_t}}(w_t^{j,\bar{z}}) \nabla f_{\bar{z}'_{j_t}}(v_{t+1}^{j,\bar{z}})\right\|$$

$$\leq \left\|w_t - w_t^{j,\bar{z}}\right\| + 2\eta_t \left\|\tilde{\nabla} g_{z_{i_t}}(w_t)\right\| \left\|\nabla f_{\bar{z}_{j_t}}(v_{t+1})\right\|$$

$$\leq \left\|w_t - w_t^{j,\bar{z}}\right\| + \frac{2L_f M_g}{\mu} \eta_t.$$

Combining the above two cases, we can get that

$$\left\|w_{t+1} - w_{t+1}^{j,\bar{z}}\right\| \leq \frac{1}{\sqrt{1 - 2\rho\eta_t}} \left\|w_t - w_t^{j,\bar{z}}\right\| \mathbb{I}[j_t \neq j] + \left(\left\|w_t - w_t^{j,\bar{z}}\right\| + \frac{2L_f M_g}{\mu} \eta_t\right) \mathbb{I}[j_t = j].$$

Taking expectation over $i_t$,

$$\mathbb{E}_{i_t} \left[\left\|w_{t+1} - w_{t+1}^{j,\bar{z}}\right\|\right]$$

$$\leq \frac{1}{\sqrt{1 - 2\rho\eta_t}} \left\|w_t - w_t^{j,\bar{z}}\right\| \mathbb{E}_{i_t} \left[\mathbb{I}[j_t \neq j]\right] + \left(\left\|w_t - w_t^{j,\bar{z}}\right\| + \frac{2L_f M_g}{\mu} \eta_t\right) \mathbb{E}_{i_t} \left[\mathbb{I}[j_t = j]\right]$$

$$\leq \frac{1}{\sqrt{1 - 2\rho\eta_t}} \left\|w_t - w_t^{j,\bar{z}}\right\| + \frac{2L_f M_g}{\mu m} \eta_t.$$

Then, taking expectation over $A$ and taking summation from $t = 1$ to $T - 1$ to get that

$$\mathbb{E}_A \left[\left\|w_T - w_T^{j,\bar{z}}\right\|\right] \leq \frac{1}{\sqrt{1 - 2\rho\eta_t}} \mathbb{E}_A \left[\left\|w_{T-1} - w_{T-1}^{j,\bar{z}}\right\|\right] + \frac{2L_f M_g}{\mu m} \eta_t$$

$$\leq \sum_{t=1}^{T-1} \left(\prod_{t'=t+1}^{T-1} \frac{1}{\sqrt{1 - 2\rho\eta_{t'}}}\right) \frac{2L_f M_g}{\mu m} \eta_t$$

$$\leq \sum_{t=1}^{T-1} \left(\prod_{t'=t+1}^{T-1} \sqrt{1 + \frac{1}{t'-1}}\right) \frac{L_f M_g}{\rho\mu m} \frac{1}{t}$$

$$\leq \sum_{t=1}^{T-1} \left(\prod_{t'=2}^{T-1} \sqrt{1 + \frac{1}{t'-1}}\right) \frac{L_f M_g}{\rho\mu m} \frac{1}{t}$$

$$\leq \sqrt{\prod_{t'=2}^{T-1} \exp\left\{\frac{1}{t'-1}\right\}} \sum_{t=1}^{T-1} \frac{L_f M_g}{\rho\mu m} \frac{1}{t}$$

$$\leq \sqrt{\exp\left\{\sum_{t'=2}^{T-1} \frac{1}{t'-1}\right\}} \frac{L_f M_g}{\rho \mu m} \sum_{t=1}^{T-1} \frac{1}{t}$$

$$\leq \frac{L_f M_g (eT)^{\frac{1}{2}} \log(eT)}{\rho \mu m}, \tag{26}$$

where the fourth inequality is from $e^x \geq 1 + x$.

Next, we will study the optimization bound. According to Equation (6) in Lemma 4, $\nabla f(w_t) = \beta \nabla g(w_t) \nabla f(v_{t+1})$, second-order Taylor expansion, Lemmas 5 and 6, we provide that, for any $p \geq \sqrt{2}$,

$$\mathbb{E}\left[F_S(w_{t+1}) - F_S(w_t)\right]$$

$$\leq \mathbb{E}\left[\langle w_{t+1} - w_t, \nabla F_S(w_t)\rangle + \frac{1}{2}\alpha \|w_{t+1} - w_t\|^2\right]$$

$$= \mathbb{E}\left[\left\langle -\eta_t \tilde{\nabla} g_{z_{i_t}}(w_t) \nabla f_{\bar{z}_{j_t}}(v_{t+1}), \nabla F_S(w_t)\right\rangle + \frac{1}{2}\alpha \left\|\eta_t \tilde{\nabla} g_{z_{i_t}}(w_t) \nabla f_{\bar{z}_{j_t}}(v_{t+1})\right\|^2\right]$$

$$= \mathbb{E}\left[\left\langle -\eta_t \left(\tilde{\nabla} g_{z_{i_t}}(w_t) - \left(p + \frac{1}{2}\right)\beta \nabla g_{z_{i_t}}(w_t) + \left(p + \frac{1}{2}\right)\beta \nabla g_{z_{i_t}}(w_t)\right) \nabla f_{\bar{z}_{j_t}}(v_{t+1}),\right.\right.$$

$$\left.\left. \nabla F_S(w_t)\right\rangle + \frac{1}{2}\alpha \left\|\eta_t \left(\tilde{\nabla} g_{z_{i_t}}(w_t) - \beta \nabla g_{z_{i_t}}(w_t) + \beta \nabla g_{z_{i_t}}(w_t)\right) \nabla f_{\bar{z}_{j_t}}(v_{t+1})\right\|^2\right]$$

$$\leq \mathbb{E}\left[-\eta_t \left\langle \left(\tilde{\nabla} g_{z_{i_t}}(w_t) - \left(p + \frac{1}{2}\right)\beta \nabla g_{z_{i_t}}(w_t)\right) \nabla f_{\bar{z}_{j_t}}(v_{t+1}), \nabla F_S(w_t)\right\rangle\right.$$

$$\left. - \left(p + \frac{1}{2}\right)\eta_t \|\nabla F_S(w_t)\|^2 + \alpha \eta_t^2 \left\|\left(\tilde{\nabla} g_{z_{i_t}}(w_t) - \beta \nabla g_{z_{i_t}}(w_t)\right) \nabla f_{\bar{z}_{j_t}}(v_{t+1})\right\|^2\right.$$

$$\left. + \alpha \eta_t^2 \|\nabla F_S(w_t)\|^2\right]$$

$$\leq \mathbb{E}\left[\frac{1}{2}\eta_t \left\|\left(\tilde{\nabla} g_{z_{i_t}}(w_t) - \left(p + \frac{1}{2}\right)\beta \nabla g_{z_{i_t}}(w_t)\right) \nabla f_{\bar{z}_{j_t}}(v_{t+1})\right\|^2 + \frac{1}{2}\eta_t \|\nabla F_S(w_t)\|^2\right.$$

$$\left. - \left(p + \frac{1}{2}\right)\eta_t \|\nabla F_S(w_t)\|^2 + \alpha \eta_t^2 \left\|\left(\tilde{\nabla} g_{z_{i_t}}(w_t) - \beta \nabla g_{z_{i_t}}(w_t)\right) \nabla f_{\bar{z}_{j_t}}(v_{t+1})\right\|^2\right.$$

$$\left. + \alpha \eta_t^2 \|\nabla F_S(w_t)\|^2\right]$$

$$= \frac{1}{2}\eta_t \mathbb{E}\left[\left\|\left(\tilde{\nabla} g_{z_{i_t}}(w_t) - \left(p + \frac{1}{2}\right)\beta \nabla g_{z_{i_t}}(w_t)\right) \nabla f_{\bar{z}_{j_t}}(v_{t+1})\right\|^2\right]$$

$$+ \alpha \eta_t^2 \mathbb{E}\left[\left\|\left(\tilde{\nabla} g_{z_{i_t}}(w_t) - \beta \nabla g_{z_{i_t}}(w_t)\right) \nabla f_{\bar{z}_{j_t}}(v_{t+1})\right\|^2\right] + \left(\alpha \eta_t^2 - p\eta_t\right) \mathbb{E}\left[\|\nabla F_S(w_t)\|^2\right]$$

$$= \frac{1}{2}\eta_t \mathbb{E}\left[\left\|\left(\frac{1}{b}\sum_{l=1}^{b} \frac{u_{t,l}}{\mu}\left(g_{z_{i_t}}(w_t + \mu u_{t,l}) - g_{z_{i_t}}(w_t)\right) - \left(p + \frac{1}{2}\right)\beta \nabla g_{z_{i_t}}(w_t)\right) \nabla f_{\bar{z}_{j_t}}(v_{t+1})\right\|^2\right]$$

$$+ \alpha \eta_t^2 \mathbb{E}\left[\left\|\left(\frac{1}{b}\sum_{l=1}^{b} \frac{u_{t,l}}{\mu}\left(g_{z_{i_t}}(w_t + \mu u_{t,l}) - g_{z_{i_t}}(w_t)\right) - \beta \nabla g_{z_{i_t}}(w_t)\right) \nabla f_{\bar{z}_{j_t}}(v_{t+1})\right\|^2\right]$$

$$+ \left(\alpha \eta_t^2 - p\eta_t\right) \mathbb{E}\left[\|\nabla F_S(w_t)\|^2\right]$$

$$= \frac{1}{2}\eta_t \mathbb{E}\left[\left\|\left(\frac{1}{b}\sum_{l=1}^{b}\left(\langle \nabla g_{z_{i_t}}(w_t), u_{t,l}\rangle u_{t,l} + \left(\frac{\mu}{2}(u_{t,l})^\top \nabla^2 g_{z_{i_t}}(w)|_{w=w_t^*} u_{t,l}\right) u_{t,l}\right)\right.\right.$$

$$\left.\left. - \left(p + \frac{1}{2}\right)\beta \nabla g_{z_{i_t}}(w_t)\right) \nabla f_{\bar{z}_{j_t}}(v_{t+1})\right\|^2\right] + \alpha \eta_t^2 \mathbb{E}\left[\left\|\left(\frac{1}{b}\sum_{l=1}^{b}\left(\langle \nabla g_{z_{i_t}}(w_t), u_{t,l}\rangle u_{t,l}\right.\right.\right.\right.$$

$$
+ \left( \frac{\mu}{2} (u_{t,l})^\top \nabla^2 g_{z_{i_t}}(w)|_{w=w_t^*} u_{t,l} \right) u_{t,l} \right) - \beta \nabla g_{z_{i_t}}(v_{t+1}) \Big) \nabla f_{\bar{z}_{j_t}}(v_{t+1}) \Big\|^2 \right]
$$

$$
+ \left( \alpha \eta_t^2 - p\eta_t \right) \mathbb{E} \left[ \|\nabla F_S(w_t)\|^2 \right]
$$

$$
\leq \left( 2\alpha \eta_t^2 + \eta_t \right) L_f^2 \mathbb{E} \left[ \left\| \frac{1}{b} \sum_{l=1}^b \left( \frac{\mu}{2} (u_{t,l})^\top \nabla^2 g_{z_{i_t}}(w)|_{w=w_t^*} u_{t,l} \right) u_{t,l} \right\|^2 \right]
$$

$$
+ L_f^2 \eta_t \mathbb{E} \left[ \left\| \frac{1}{b} \sum_{l=1}^b \langle \nabla g_{z_{i_t}}(w_t), u_{t,l} \rangle u_{t,l} - \left( p + \frac{1}{2} \right) \beta \nabla g_{z_{i_t}}(w_t) \right\|^2 \right]
$$

$$
+ 2\alpha L_f^2 \eta_t^2 \mathbb{E} \left[ \left\| \frac{1}{b} \sum_{l=1}^b \langle \nabla g_{z_{i_t}}(w_t), u_{t,l} \rangle u_{t,l} - \beta \nabla g_{z_{i_t}}(w_t) \right\|^2 \right]
$$

$$
+ \left( \alpha \eta_t^2 - p\eta_t \right) \mathbb{E} \left[ \|\nabla F_S(w_t)\|^2 \right]
$$

$$
\leq \left( \alpha \eta_t^2 - p\eta_t \right) \mathbb{E} \left[ \|\nabla F_S(w_t)\|^2 \right] + \frac{L_f^2 \mu^2 \alpha_g^2}{4} (2\alpha \eta_t^2 + \eta_t) \mathbb{E} \left[ \|u_{t,1}\|^6 \right]
$$

$$
+ \left( \frac{d - 2\beta + \beta^2}{b} 2\alpha L_f^2 \eta_t^2 + \frac{d - (2p+1)\beta + \left(p + \frac{1}{2}\right)^2 \beta^2}{b} L_f^2 \eta_t \right) \mathbb{E} \left[ \|\nabla g_{z_{i_t}}(w_t)\|^2 \right]
$$

$$
\leq \left( \alpha \eta_t^2 - p\eta_t \right) \mathbb{E} \left[ \|\nabla F_S(w_t)\|^2 \right] + \frac{d L_f^2 \mu^2 \alpha_g^2}{4(d+6)} (2\alpha \eta_t^2 + \eta_t)
$$

$$
+ \frac{d - 2\beta + \beta^2}{b} 2\alpha L_g^2 L_f^2 \eta_t^2 + \frac{d - (2p+1)\beta + \left(p + \frac{1}{2}\right)^2 \beta^2}{b} L_g^2 L_f^2 \eta_t.
$$

Let $d_1 = d - 2\beta + \beta^2$ and $d_2 = d - (2p+1)\beta + \left(p + \frac{1}{2}\right)^2 \beta^2$ to get that

$$
\mathbb{E} \left[ F_S(w_{t+1}) - F_S(w_t) \right]
$$

$$
\leq \left( \alpha \eta_t^2 - p\eta_t \right) \mathbb{E} \left[ \|\nabla F_S(w_t)\|^2 \right] + \frac{d L_f^2 \mu^2 \alpha_g^2}{4(d+6)} (2\alpha \eta_t^2 + \eta_t) + \frac{L_g^2 L_f^2}{b} \left( 2\alpha d_1 \eta_t^2 + d_2 \eta_t \right)
$$

$$
\leq -\frac{1}{2} p\eta_t \mathbb{E} \left[ \|\nabla F_S(w_t)\|^2 \right] + \frac{d L_f^2 \mu^2 \alpha_g^2}{4(d+6)} (2\alpha \eta_t^2 + \eta_t) + \frac{L_g^2 L_f^2}{b} \left( 2\alpha d_1 \eta_t^2 + d_2 \eta_t \right)
$$

$$
\leq -p\gamma \eta_t \mathbb{E} \left[ F_S(w_t) - F_S(w(S)) \right] + \frac{d L_f^2 \mu^2 \alpha_g^2}{4(d+6)} (2\alpha \eta_t^2 + \eta_t) + \frac{L_g^2 L_f^2}{b} \left( 2\alpha d_1 \eta_t^2 + d_2 \eta_t \right),
$$

where the second inequality is due to $\eta_t = \frac{1}{p\gamma t} \leq \frac{p}{2\alpha t} \leq \frac{p}{2\alpha}$ when $p \geq \sqrt{\frac{2\alpha}{\gamma}}$, and the last inequality is from Equation 8 in Lemma 4. Then,

$$
\mathbb{E} \left[ F_S(w_{t+1}) - F_S(w(S)) \right]
$$

$$
\leq (1 - p\gamma \eta_t) \mathbb{E} \left[ F_S(w_t) - F_S(w(S)) \right] + \frac{d L_f^2 \mu^2 \alpha_g^2}{4(d+6)} (2\alpha \eta_t^2 + \eta_t) + \frac{L_g^2 L_f^2}{b} \left( 2\alpha d_1 \eta_t^2 + d_2 \eta_t \right)
$$

$$
= \left( 1 - \frac{1}{t} \right) \mathbb{E} \left[ F_S(w_t) - F_S(w(S)) \right] + \frac{d L_f^2 \mu^2 \alpha_g^2}{4(d+6)} \left( \frac{2}{p^2 \alpha t^2} + \frac{1}{p\alpha t} \right) + \frac{L_g^2 L_f^2}{b} \left( \frac{2d_1}{p^2 \alpha t^2} + \frac{d_2}{p\alpha t} \right).
$$

We multiply both sides of the above inequality by $t$ to get that

$$
t\mathbb{E} \left[ F_S(w_{t+1}) - F_S(w(S)) \right]
$$

$$
\leq (t-1)\mathbb{E} \left[ F_S(w_t) - F_S(w(S)) \right] + \frac{d L_f^2 \mu^2 \alpha_g^2}{4(d+6)} \left( \frac{2}{p^2 \alpha t} + \frac{1}{p\alpha} \right) + \frac{L_g^2 L_f^2}{b} \left( \frac{2d_1}{p^2 \alpha t} + \frac{d_2}{p\alpha} \right).
$$

Then

$$
(T-1)\mathbb{E} \left[ F_S(w_T) - F_S(w(S)) \right]
$$

$$\leq (T-2)\mathbb{E}\left[F_S(w_{T-1}) - F_S(w(S))\right] + \frac{dL_f^2\mu^2\alpha_g^2}{4(d+6)}\left(\frac{2}{p^2\alpha(T-1)} + \frac{1}{p\alpha}\right) + \frac{L_g^2L_f^2}{b}\left(\frac{2d_1}{p^2\alpha(T-1)} + \frac{d_2}{p\alpha}\right)$$

$$\leq \frac{dL_f^2\mu^2\alpha_g^2}{4(d+6)}\left(\sum_{t=1}^{T-1}\frac{2}{p^2\alpha t} + \frac{T-1}{p\alpha}\right) + \frac{L_g^2L_f^2}{b}\left(\sum_{t=1}^{T-1}\frac{2d_1}{p^2\alpha t} + \frac{d_2(T-1)}{p\alpha}\right)$$

$$\leq \frac{dL_f^2\mu^2\alpha_g^2}{4(d+6)}\left(\frac{2\log(eT)}{p^2\alpha} + \frac{T-1}{p\alpha}\right) + \frac{L_g^2L_f^2}{b}\left(\frac{2d_1\log(eT)}{p^2\alpha} + \frac{d_2(T-1)}{p\alpha}\right).$$

That is

$$\mathbb{E}\left[F_S(w_T) - F_S(w(S))\right]$$
$$\leq \frac{dL_f^2\mu^2\alpha_g^2}{4(d+6)}\left(\frac{2\log(eT)}{p^2\alpha(T-1)} + \frac{1}{p\alpha}\right) + \frac{L_g^2L_f^2}{b}\left(\frac{2d_1\log(eT)}{p^2\alpha(T-1)} + \frac{d_2}{p\alpha}\right). \tag{27}$$

Combining Theorem 1, Equations (24), (26) and (27), we can get that

$$\mathbb{E}\left[F(w_T) - F(w^*)\right] \leq \mathcal{O}\left(\left(n^{-1} + m^{-1}\right)T^{\frac{1}{2}}\log T + n^{-\frac{1}{2}} + \mu^2 + b^{-1}d_2\right).$$

**SCSC:** Similar to the stability proof of SCGD except for

$$\nabla^2 \hat{f}_{z_{i_t}, \bar{z}_{j_t}}(w_t)$$
$$= \frac{1}{b}\sum_{l=1}^{b}\frac{u_{t,l}}{\mu}\left(\nabla g_{z_{i_t}}(w_t + \mu u_{t,l}) - \nabla g_{z_{i_t}}(w_t)\right)\nabla f_{\bar{z}_{j_t}}(v_{t+1})$$
$$+ \frac{1}{b}\sum_{l=1}^{b}\frac{u_{t,l}}{\mu}\left(g_{z_{i_t}}(w_t + \mu u_{t,l}) - g_{z_{i_t}}(w_t)\right)\nabla^2 f_{\bar{z}_{j_t}}(v_{t+1})\nabla g_{z_{i_t}}^\top(w_t).$$

based on the update of SCSC, we have that, for $\eta_t \leq \frac{1}{2\rho t} \leq \frac{3}{2\rho}, \rho = \frac{1}{\mu}\left(L_g\alpha_f M_g + M_g' L_f\right)$,

$$\mathbb{E}_A\left[\left\|w_T - w_T^{i,z}\right\|\right] \leq \frac{L_f M_g(eT)^{\frac{1}{2}}\log(eT)}{\rho\mu n} \tag{28}$$

and

$$\mathbb{E}_A\left[\left\|w_T - w_T^{j,\bar{z}}\right\|\right] \leq \frac{L_f M_g(eT)^{\frac{1}{2}}\log(eT)}{\rho\mu m}. \tag{29}$$

Similar to the optimization proof of SCGD, we also have that

$$\mathbb{E}\left[F_S(w_T) - F_S(w(S))\right]$$
$$\leq \frac{dL_f^2\mu^2\alpha_g^2}{4(d+6)}\left(\frac{2\log(eT)}{p^2\alpha(T-1)} + \frac{1}{p\alpha}\right) + \frac{L_g^2L_f^2}{b}\left(\frac{2d_1\log(eT)}{p^2\alpha(T-1)} + \frac{d_2}{p\alpha}\right), \tag{30}$$

where $d_1 = d - 1, d_2 = d - (2p+1) + \left(p + \frac{1}{2}\right)^2$. Combining Theorem 1, Equations (28), (29) and (30), we can get that

$$\mathbb{E}\left[F(w_T) - F(w^*)\right] \leq \mathcal{O}\left(\left(n^{-1} + m^{-1}\right)T^{\frac{1}{2}}\log T + n^{-\frac{1}{2}} + \mu^2 + b^{-1}d_2\right).$$

$\square$

**Proof of Corollary 2**:

**SCGD: 1)** $\mathbb{E}_A\left[\left\|w_T - w_T^{i,z}\right\|\right]$: Firstly, when $i_t \neq i$, there holds

$$\left\|w_{t+1} - w_{t+1}^{i,z}\right\|^2$$
$$= \left\|w_t - w_t^{i,z} - \eta_t\tilde{\nabla}g_{z_{i_t}}(w_t)\tilde{\nabla}f_{\bar{z}_{j_t}}(v_{t+1}) + \eta_t\tilde{\nabla}g_{z_{i_t}}(w_t^{i,z})\tilde{\nabla}f_{\bar{z}_{j_t}}(v_{t+1}^{i,z})\right\|^2$$

$$= \left\| w_t - w_t^{i,z} \right\|^2 + \eta_t^2 \left\| \tilde{\nabla} g_{z_{i_t}}(w_t) \tilde{\nabla} f_{\bar{z}_{j_t}}(v_{t+1}) - \tilde{\nabla} g_{z_{i_t}}(w_t^{i,z}) \tilde{\nabla} f_{\bar{z}_{j_t}}(v_{t+1}^{i,z}) \right\|^2$$

$$- 2\eta_t \left\langle w_t - w_t^{i,z}, \tilde{\nabla} g_{z_{i_t}}(w_t) \tilde{\nabla} f_{\bar{z}_{j_t}}(v_{t+1}) - \tilde{\nabla} g_{z_{i_t}}(w_t^{i,z}) \tilde{\nabla} f_{\bar{z}_{j_t}}(v_{t+1}^{i,z}) \right\rangle$$

$$= \left\| w_t - w_t^{i,z} \right\|^2 + \eta_t^2 \left\| \frac{1}{b^2} \sum_{l=1}^{b} \frac{u_{t,l}}{\mu} \left( f_{\bar{z}_{j_t}}(v_{t+1} + \mu u_{t,l}) - f_{\bar{z}_{j_t}}(v_{t+1}) \right) \right.$$

$$\sum_{l=1}^{b} \frac{u_{t,l}}{\mu} \left( g_{z_{i_t}}(w_t + \mu u_{t,l}) - g_{z_{i_t}}(w_t) \right) - \frac{1}{b^2} \sum_{l=1}^{b} \frac{u_{t,l}}{\mu} \left( f_{\bar{z}_{j_t}}(v_{t+1}^{i,z} + \mu u_{t,l}) - f_{\bar{z}_{j_t}}(v_{t+1}^{i,z}) \right)$$

$$\left. \sum_{l=1}^{b} \frac{u_{t,l}}{\mu} \left( g_{z_{i_t}}(w_t^{i,z} + \mu u_{t,l}) - g_{z_{i_t}}(w_t^{i,z}) \right) \right\|^2 - 2\eta_t \left\langle w_t - w_t^{i,z}, \frac{1}{b^2} \sum_{l=1}^{b} \frac{u_{t,l}}{\mu} \right.$$

$$\left( f_{\bar{z}_{j_t}}(v_{t+1} + \mu u_{t,l}) - f_{\bar{z}_{j_t}}(v_{t+1}) \right) \sum_{l=1}^{b} \frac{u_{t,l}}{\mu} \left( g_{z_{i_t}}(w_t + \mu u_{t,l}) - g_{z_{i_t}}(w_t) \right)$$

$$\left. - \frac{1}{b^2} \sum_{l=1}^{b} \frac{u_{t,l}}{\mu} \left( f_{\bar{z}_{j_t}}(v_{t+1}^{i,z} + \mu u_{t,l}) - f_{\bar{z}_{j_t}}(v_{t+1}^{i,z}) \right) \sum_{l=1}^{b} \frac{u_{t,l}}{\mu} \left( g_{z_{i_t}}(w_t^{i,z} + \mu u_{t,l}) - g_{z_{i_t}}(w_t^{i,z}) \right) \right\rangle .$$

With Assumption 3, the terms $f(v_{t+1})$ and $g(w_t)$ are both differentiable. Thus, $\frac{1}{b^2} \sum_{l=1}^{b} \frac{u_{t,l}}{\mu} \left( f(v_{t+1} + \mu u_{t,l}) - f(v_{t+1}) \right) \sum_{l=1}^{b} \frac{u_{t,l}}{\mu} \left( g(w_t + \mu u_{t,l}) - g(w_t) \right)$ is also differentiable. It is reasonable to assume that there exists a primitive function $\hat{f}(w_t)$ at least whose derivative function is $\frac{1}{b^2} \sum_{l=1}^{b} \frac{u_{t,l}}{\mu} \left( f(v_{t+1} + \mu u_{t,l}) - f(v_{t+1}) \right) \sum_{l=1}^{b} \frac{u_{t,l}}{\mu} \left( g(w_t + \mu u_{t,l}) - g(w_t) \right)$. Then

$$\left\| w_{t+1} - w_{t+1}^{i,z} \right\|^2$$

$$= \left\| w_t - w_t^{i,z} \right\|^2 + \eta_t^2 \left\| \nabla \hat{f}_{z_{i_t}, \bar{z}_{j_t}}(w_t) \nabla \hat{f}_{z_{i_t} \bar{z}_{j_t}}(w_t^{i,z}) \right\|^2$$

$$- 2\eta_t \left\langle w_t - w_t^{i,z}, \nabla \hat{f}_{z_{i_t}, \bar{z}_{j_t}}(w_t) - \nabla \hat{f}_{z_{i_t} \bar{z}_{j_t}}(w_t^{i,z}) \right\rangle . \tag{31}$$

Taking derivative of $\nabla \hat{f}_{z_{i_t}, \bar{z}_{j_t}}(w_t)$ over $w_t$, we get that

$$\nabla^2 \hat{f}_{z_{i_t}, \bar{z}_{j_t}}(w_t)$$

$$= \frac{\beta}{b^2} \sum_{l=1}^{b} \frac{u_{t,l}}{\mu} \nabla g_{z_{i_t}}(w_t) \left( \nabla f_{\bar{z}_{j_t}}(v_{t+1} + \mu u_{t,l}) - \nabla f_{\bar{z}_{j_t}}(v_{t+1}) \right) \sum_{l=1}^{b} \frac{u_{t,l}}{\mu} \left( g_{z_{i_t}}(w_t + \mu u_{t,l}) - g_{z_{i_t}}(w_t) \right)$$

$$+ \frac{1}{b^2} \sum_{l=1}^{b} \frac{u_{t,l}}{\mu} \left( f_{\bar{z}_{j_t}}(v_{t+1} + \mu u_{t,l}) - f_{\bar{z}_{j_t}}(v_{t+1}) \right) \sum_{l=1}^{b} \frac{u_{t,l}}{\mu} \left( \nabla g_{z_{i_t}}(w_t + \mu u_{t,l}) - \nabla g_{z_{i_t}}(w_t) \right) .$$

Thus,

$$\left\| \nabla^2 \hat{f}_{z_{i_t}, \bar{z}_{j_t}}(w_t) \right\|$$

$$= \left\| \frac{\beta}{b^2} \sum_{l=1}^{b} \frac{u_{t,l}}{\mu} \nabla g_{z_{i_t}}(w_t) \left( \nabla f_{\bar{z}_{j_t}}(v_{t+1} + \mu u_{t,l}) - \nabla f_{\bar{z}_{j_t}}(v_{t+1}) \right) \sum_{l=1}^{b} \frac{u_{t,l}}{\mu} \left( g_{z_{i_t}}(w_t + \mu u_{t,l}) - g_{z_{i_t}}(w_t) \right) \right.$$

$$\left. + \frac{1}{b^2} \sum_{l=1}^{b} \frac{u_{t,l}}{\mu} \left( f_{\bar{z}_{j_t}}(v_{t+1} + \mu u_{t,l}) - f_{\bar{z}_{j_t}}(v_{t+1}) \right) \sum_{l=1}^{b} \frac{u_{t,l}}{\mu} \left( \nabla g_{z_{i_t}}(w_t + \mu u_{t,l}) - \nabla g_{z_{i_t}}(w_t) \right) \right\|$$

$$\leq \left\| \frac{\beta}{b^2} \sum_{l=1}^{b} \frac{u_{t,l}}{\mu} \nabla g_{z_{i_t}}(w_t) \left( \nabla f_{\bar{z}_{j_t}}(v_{t+1} + \mu u_{t,l}) - \nabla f_{\bar{z}_{j_t}}(v_{t+1}) \right) \sum_{l=1}^{b} \frac{u_{t,l}}{\mu} \left( g_{z_{i_t}}(w_t + \mu u_{t,l}) - g_{z_{i_t}}(w_t) \right) \right\|$$

$$+ \left\| \frac{1}{b^2} \sum_{l=1}^{b} \frac{u_{t,l}}{\mu} \left( f_{\bar{z}_{j_t}}(v_{t+1} + \mu u_{t,l}) - f_{\bar{z}_{j_t}}(v_{t+1}) \right) \sum_{l=1}^{b} \frac{u_{t,l}}{\mu} \left( \nabla g_{z_{i_t}}(w_t + \mu u_{t,l}) - \nabla g_{z_{i_t}}(w_t) \right) \right\|$$

$$\leq \frac{1}{\mu^2}\left(\beta L_g M'_f M_g + M_f M'_g\right).$$

Let $\rho = \frac{1}{\mu^2}\left(\beta L_g M'_f M_g + M_f M'_g\right)$, then $\hat{f}_{z_{i_t}, \bar{z}_{j_t}}(w_t)$ is $\rho$-smooth. And we can know that $\lambda_{\min}\left(\nabla^2 \hat{f}_{z_{i_t}, \bar{z}_{j_t}}(w_t)\right) \geq -\left\|\nabla^2 \hat{f}_{z_{i_t}, \bar{z}_{j_t}}(w_t)\right\| \geq -\rho$. According to Lemma 3, we can get that

$$\left\langle w_t - w_t^{i,z}, \nabla \hat{f}_{z_{i_t}, \bar{z}_{j_t}}(w_t) - \nabla \hat{f}_{z_{i_t} \bar{z}_{j_t}}(w_t^{i,z})\right\rangle$$

$$\geq 2\eta_t \left(1 - \frac{\rho\eta_t}{2}\right)\left\|\nabla \hat{f}_{z_{i_t}, \bar{z}_{j_t}}(w_t) - \nabla \hat{f}_{z_{i_t} \bar{z}_{j_t}}(w_t^{i,z})\right\|^2 - \rho\left\|w_t - w_t^{i,z} - \nabla \hat{f}_{z_{i_t}, \bar{z}_{j_t}}(w_t) + \nabla \hat{f}_{z_{i_t}, \bar{z}_{j_t}}(w_t^{i,z})\right\|^2$$

$$= 2\eta_t \left(1 - \frac{\rho\eta_t}{2}\right)\left\|\nabla \hat{f}_{z_{i_t}, \bar{z}_{j_t}}(w_t) - \nabla \hat{f}_{z_{i_t} \bar{z}_{j_t}}(w_t^{i,z})\right\|^2 - \rho\left\|w_{t+1} - w_{t+1}^{i,z}\right\|^2.$$

Now, plugging the above inequality back into Equation (31) yields

$$\left\|w_{t+1} - w_{t+1}^{i,z}\right\|^2$$

$$\leq \left\|w_t - w_t^{i,z}\right\|^2 + \left(\eta_t^2 - 4\eta_t^2\left(1 - \frac{\rho\eta_t}{2}\right)\right)\left\|\nabla \hat{f}_{z_{i_t}, \bar{z}_{j_t}}(w_t)\nabla \hat{f}_{z_{i_t}\bar{z}_{j_t}}(w_t^{i,z})\right\|^2 + 2\rho\eta_t\left\|w_{t+1} - w_{t+1}^{i,z}\right\|^2$$

$$\leq \left\|w_t - w_t^{i,z}\right\|^2 + 2\rho\eta_t\left\|w_{t+1} - w_{t+1}^{i,z}\right\|^2,$$

where the second inequality is due to $\eta_t \leq \frac{1}{2\rho t} \leq \frac{3}{2\rho}$. The above inequality implies

$$\left\|w_{t+1} - w_{t+1}^{i,z}\right\| \leq \frac{1}{\sqrt{1 - 2\rho\eta_t}}\left\|w_t - w_t^{i,z}\right\|,$$

Secondly, when $i_t = i$, there holds

$$\left\|w_{t+1} - w_{t+1}^{i,z}\right\|$$

$$= \left\|w_t - w_t^{i,z} - \eta_t \tilde{\nabla} g_{z_{i_t}}(w_t)\tilde{\nabla} f_{\bar{z}_{j_t}}(v_{t+1}) + \eta_t \tilde{\nabla} g_{z'_{i_t}}(w_t^{i,z})\tilde{\nabla} f_{\bar{z}_{j_t}}(v_{t+1}^{i,z})\right\|$$

$$\leq \left\|w_t - w_t^{i,z}\right\| + \eta_t \left\|\tilde{\nabla} g_{z_{i_t}}(w_t)\tilde{\nabla} f_{\bar{z}_{j_t}}(v_{t+1}) - \tilde{\nabla} g_{z'_{i_t}}(w_t^{i,z})\tilde{\nabla} f_{\bar{z}_{j_t}}(v_{t+1}^{i,z})\right\|$$

$$\leq \left\|w_t - w_t^{i,z}\right\| + 2\eta_t \left\|\tilde{\nabla} g_{z_{i_t}}(w_t)\right\|\left\|\tilde{\nabla} f_{\bar{z}_{j_t}}(v_{t+1})\right\|$$

$$\leq \left\|w_t - w_t^{i,z}\right\| + \frac{2M_f M_g}{\mu^2}\eta_t.$$

Combining the above two cases, we can get that

$$\left\|w_{t+1} - w_{t+1}^{i,z}\right\| \leq \frac{1}{\sqrt{1 - 2\rho\eta_t}}\left\|w_t - w_t^{i,z}\right\|\mathbb{I}[i_t \neq i] + \left(\left\|w_t - w_t^{i,z}\right\| + \frac{2M_f M_g}{\mu^2}\eta_t\right)\mathbb{I}[i_t = i].$$

Taking expectation over $i_t$,

$$\mathbb{E}_{i_t}\left[\left\|w_{t+1} - w_{t+1}^{i,z}\right\|\right]$$

$$\leq \frac{1}{\sqrt{1 - 2\rho\eta_t}}\left\|w_t - w_t^{i,z}\right\|\mathbb{E}_{i_t}[\mathbb{I}[i_t \neq i]] + \left(\left\|w_t - w_t^{i,z}\right\| + \frac{2M_f M_g}{\mu^2}\eta_t\right)\mathbb{E}_{i_t}[\mathbb{I}[i_t = i]]$$

$$\leq \frac{1}{\sqrt{1 - 2\rho\eta_t}}\left\|w_t - w_t^{i,z}\right\| + \frac{2M_f M_g}{\mu^2 n}\eta_t.$$

Then, taking expectation over $A$ and taking summation from $t = 1$ to $T - 1$ to get that

$$\mathbb{E}_A\left[\left\|w_T - w_T^{i,z}\right\|\right] \leq \frac{1}{\sqrt{1 - 2\rho\eta_t}}\mathbb{E}_A\left[\left\|w_{T-1} - w_{T-1}^{i,z}\right\|\right] + \frac{2M_f M_g}{\mu^2 n}\eta_t$$

$$\leq \sum_{t=1}^{T-1}\left(\prod_{t'=t+1}^{T-1}\frac{1}{\sqrt{1 - 2\rho\eta_{t'}}}\right)\frac{2M_f M_g}{\mu^2 n}\eta_t$$

$$\leq \sum_{t=1}^{T-1} \left( \prod_{t'=t+1}^{T-1} \sqrt{1+\frac{1}{t'-1}} \right) \frac{M_f M_g}{\rho \mu^2 n} \frac{1}{t}$$

$$\leq \sum_{t=1}^{T-1} \left( \prod_{t'=2}^{T-1} \sqrt{1+\frac{1}{t'-1}} \right) \frac{M_f M_g}{\rho \mu^2 n} \frac{1}{t}$$

$$\leq \sqrt{\prod_{t'=2}^{T-1} \exp\left\{ \frac{1}{t'-1} \right\}} \sum_{t=1}^{T-1} \frac{M_f M_g}{\rho \mu^2 n} \frac{1}{t}$$

$$\leq \sqrt{\exp\left\{ \sum_{t'=2}^{T-1} \frac{1}{t'-1} \right\}} \frac{M_f M_g}{\rho \mu^2 n} \sum_{t=1}^{T-1} \frac{1}{t}$$

$$\leq \frac{M_f M_g (eT)^{\frac{1}{2}} \log(eT)}{\rho \mu^2 n}, \tag{32}$$

where the fourth inequality is from $e^x \geq 1 + x$.

**2)** $\mathbb{E}_A \left[ \left\| w_T - w_T^{j,\bar{z}} \right\| \right]$: Firstly, when $j_t \neq j$, there holds

$$\left\| w_{t+1} - w_{t+1}^{j,\bar{z}} \right\|^2$$

$$= \left\| w_t - w_t^{j,\bar{z}} - \eta_t \tilde{\nabla} g_{z_{i_t}}(w_t) \tilde{\nabla} f_{\bar{z}_{j_t}}(v_{t+1}) + \eta_t \tilde{\nabla} g_{z_{i_t}}(w_t^{j,\bar{z}}) \tilde{\nabla} f_{\bar{z}_{j_t}}(v_{t+1}^{j,\bar{z}}) \right\|^2$$

$$= \left\| w_t - w_t^{j,\bar{z}} \right\|^2 + \eta_t^2 \left\| \tilde{\nabla} g_{z_{i_t}}(w_t) \tilde{\nabla} f_{\bar{z}_{j_t}}(v_{t+1}) - \tilde{\nabla} g_{z_{i_t}}(w_t^{j,\bar{z}}) \tilde{\nabla} f_{\bar{z}_{j_t}}(v_{t+1}^{j,\bar{z}}) \right\|^2$$

$$- 2\eta_t \left\langle w_t - w_t^{j,\bar{z}}, \tilde{\nabla} g_{z_{i_t}}(w_t) \tilde{\nabla} f_{\bar{z}_{j_t}}(v_{t+1}) - \tilde{\nabla} g_{z_{i_t}}(w_t^{j,\bar{z}}) \tilde{\nabla} f_{\bar{z}_{j_t}}(v_{t+1}^{j,\bar{z}}) \right\rangle$$

$$= \left\| w_t - w_t^{j,\bar{z}} \right\|^2 + \eta_t^2 \left\| \frac{1}{b^2} \sum_{l=1}^{b} \frac{u_{t,l}}{\mu} \left( f_{\bar{z}_{j_t}}(v_{t+1} + \mu u_{t,l}) - f_{\bar{z}_{j_t}}(v_{t+1}) \right) \right.$$

$$\sum_{l=1}^{b} \frac{u_{t,l}}{\mu} \left( g_{z_{i_t}}(w_t + \mu u_{t,l}) - g_{z_{i_t}}(w_t) \right) - \frac{1}{b^2} \sum_{l=1}^{b} \frac{u_{t,l}}{\mu} \left( f_{\bar{z}_{j_t}}(v_{t+1}^{j,\bar{z}} + \mu u_{t,l}) - f_{\bar{z}_{j_t}}(v_{t+1}^{j,\bar{z}}) \right)$$

$$\left. \sum_{l=1}^{b} \frac{u_{t,l}}{\mu} \left( g_{z_{i_t}}(w_t^{j,\bar{z}} + \mu u_{t,l}) - g_{z_{i_t}}(w_t^{j,\bar{z}}) \right) \right\|^2 - 2\eta_t \left\langle w_t - w_t^{j,\bar{z}}, \frac{1}{b^2} \sum_{l=1}^{b} \frac{u_{t,l}}{\mu} \right.$$

$$\left( f_{\bar{z}_{j_t}}(v_{t+1} + \mu u_{t,l}) - f_{\bar{z}_{j_t}}(v_{t+1}) \right) \sum_{l=1}^{b} \frac{u_{t,l}}{\mu} \left( g_{z_{i_t}}(w_t + \mu u_{t,l}) - g_{z_{i_t}}(w_t) \right)$$

$$\left. - \frac{1}{b^2} \sum_{l=1}^{b} \frac{u_{t,l}}{\mu} \left( f_{\bar{z}_{j_t}}(v_{t+1}^{j,\bar{z}} + \mu u_{t,l}) - f_{\bar{z}_{j_t}}(v_{t+1}^{j,\bar{z}}) \right) \sum_{l=1}^{b} \frac{u_{t,l}}{\mu} \left( g_{z_{i_t}}(w_t^{j,\bar{z}} + \mu u_{t,l}) - g_{z_{i_t}}(w_t^{j,\bar{z}}) \right) \right\rangle.$$

With Assumption 3, the terms $\nabla f(v_{t+1})$ and $g(w_t)$ are both differentiable. Thus, $\frac{1}{b^2} \sum_{l=1}^{b} \frac{u_{t,l}}{\mu} \left( f(v_{t+1} + \mu u_{t,l}) - f(v_{t+1}) \right) \sum_{l=1}^{b} \frac{u_{t,l}}{\mu} \left( g(w_t + \mu u_{t,l}) - g(w_t) \right)$ is also differentiable. It is reasonable to assume that there exists a primitive function $\hat{f}(w_t)$ at least whose derivative function is $\frac{1}{b^2} \sum_{l=1}^{b} \frac{u_{t,l}}{\mu} \left( f(v_{t+1} + \mu u_{t,l}) - f(v_{t+1}) \right) \sum_{l=1}^{b} \frac{u_{t,l}}{\mu} \left( g(w_t + \mu u_{t,l}) - g(w_t) \right)$. Then

$$\left\| w_{t+1} - w_{t+1}^{j,\bar{z}} \right\|^2$$

$$= \left\| w_t - w_t^{j,\bar{z}} \right\|^2 + \eta_t^2 \left\| \nabla \hat{f}_{z_{i_t}, \bar{z}_{j_t}}(w_t) \nabla \hat{f}_{z_{i_t} \bar{z}_{j_t}}(w_t^{j,\bar{z}}) \right\|^2$$

$$- 2\eta_t \left\langle w_t - w_t^{j,\bar{z}}, \nabla \hat{f}_{z_{i_t}, \bar{z}_{j_t}}(w_t) - \nabla \hat{f}_{z_{i_t} \bar{z}_{j_t}}(w_t^{j,\bar{z}}) \right\rangle. \tag{33}$$

Taking derivative of $\nabla \hat{f}_{z_{i_t}, \bar{z}_{j_t}}(w_t)$ over $w_t$, we get that

$$\nabla^2 \hat{f}_{z_{i_t}, \bar{z}_{j_t}}(w_t)$$

$$=\frac{\beta}{b^2}\sum_{l=1}^{b}\frac{u_{t,l}}{\mu}\nabla g_{z_{i_t}}(w_t)\left(\nabla f_{\bar{z}_{j_t}}(v_{t+1}+\mu u_{t,l})-\nabla f_{\bar{z}_{j_t}}(v_{t+1})\right)\sum_{l=1}^{b}\frac{u_{t,l}}{\mu}\left(g_{z_{i_t}}(w_t+\mu u_{t,l})-g_{z_{i_t}}(w_t)\right)$$

$$+\frac{1}{b^2}\sum_{l=1}^{b}\frac{u_{t,l}}{\mu}\left(f_{\bar{z}_{j_t}}(v_{t+1}+\mu u_{t,l})-f_{\bar{z}_{j_t}}(v_{t+1})\right)\sum_{l=1}^{b}\frac{u_{t,l}}{\mu}\left(\nabla g_{z_{i_t}}(w_t+\mu u_{t,l})-\nabla g_{z_{i_t}}(w_t)\right).$$

Thus,

$$\left\|\nabla^2\hat{f}_{z_{i_t},\bar{z}_{j_t}}(w_t)\right\|$$

$$=\left\|\frac{\beta}{b^2}\sum_{l=1}^{b}\frac{u_{t,l}}{\mu}\nabla g_{z_{i_t}}(w_t)\left(\nabla f_{\bar{z}_{j_t}}(v_{t+1}+\mu u_{t,l})-\nabla f_{\bar{z}_{j_t}}(v_{t+1})\right)\sum_{l=1}^{b}\frac{u_{t,l}}{\mu}\left(g_{z_{i_t}}(w_t+\mu u_{t,l})-g_{z_{i_t}}(w_t)\right)\right.$$

$$\left.+\frac{1}{b^2}\sum_{l=1}^{b}\frac{u_{t,l}}{\mu}\left(f_{\bar{z}_{j_t}}(v_{t+1}+\mu u_{t,l})-f_{\bar{z}_{j_t}}(v_{t+1})\right)\sum_{l=1}^{b}\frac{u_{t,l}}{\mu}\left(\nabla g_{z_{i_t}}(w_t+\mu u_{t,l})-\nabla g_{z_{i_t}}(w_t)\right)\right\|$$

$$\leq\left\|\frac{\beta}{b^2}\sum_{l=1}^{b}\frac{u_{t,l}}{\mu}\nabla g_{z_{i_t}}(w_t)\left(\nabla f_{\bar{z}_{j_t}}(v_{t+1}+\mu u_{t,l})-\nabla f_{\bar{z}_{j_t}}(v_{t+1})\right)\sum_{l=1}^{b}\frac{u_{t,l}}{\mu}\left(g_{z_{i_t}}(w_t+\mu u_{t,l})-g_{z_{i_t}}(w_t)\right)\right\|$$

$$+\left\|\frac{1}{b^2}\sum_{l=1}^{b}\frac{u_{t,l}}{\mu}\left(f_{\bar{z}_{j_t}}(v_{t+1}+\mu u_{t,l})-f_{\bar{z}_{j_t}}(v_{t+1})\right)\sum_{l=1}^{b}\frac{u_{t,l}}{\mu}\left(\nabla g_{z_{i_t}}(w_t+\mu u_{t,l})-\nabla g_{z_{i_t}}(w_t)\right)\right\|$$

$$\leq\frac{1}{\mu^2}\left(\beta L_g M_f' M_g+M_f M_g'\right).$$

Let $\rho=\frac{1}{\mu^2}\left(\beta L_g M_f' M_g+M_f M_g'\right)$, then $\hat{f}_{z_{i_t},\bar{z}_{j_t}}(w_t)$ is $\rho$-smooth. And we can know that $\lambda_{\min}\left(\nabla^2\hat{f}_{z_{i_t},\bar{z}_{j_t}}(w_t)\right)\geq-\left\|\nabla^2\hat{f}_{z_{i_t},\bar{z}_{j_t}}(w_t)\right\|\geq-\rho$. According to Lemma 3, we can get that

$$\left\langle w_t-w_t^{j,\bar{z}},\nabla\hat{f}_{z_{i_t},\bar{z}_{j_t}}(w_t)-\nabla\hat{f}_{z_{i_t}\bar{z}_{j_t}}(w_t^{j,\bar{z}})\right\rangle$$

$$\geq 2\eta_t\left(1-\frac{\rho\eta_t}{2}\right)\left\|\nabla\hat{f}_{z_{i_t},\bar{z}_{j_t}}(w_t)-\nabla\hat{f}_{z_{i_t}\bar{z}_{j_t}}(w_t^{j,\bar{z}})\right\|^2-\rho\left\|w_t-w_t^{j,\bar{z}}-\nabla\hat{f}_{z_{i_t},\bar{z}_{j_t}}(w_t)+\nabla\hat{f}_{z_{i_t}\bar{z}_{j_t}}(w_t^{j,\bar{z}})\right\|^2$$

$$=2\eta_t\left(1-\frac{\rho\eta_t}{2}\right)\left\|\nabla\hat{f}_{z_{i_t},\bar{z}_{j_t}}(w_t)-\nabla\hat{f}_{z_{i_t}\bar{z}_{j_t}}(w_t^{j,\bar{z}})\right\|^2-\rho\left\|w_{t+1}-w_{t+1}^{j,\bar{z}}\right\|^2.$$

Now, plugging the above inequality back into Equation (33) yields

$$\left\|w_{t+1}-w_{t+1}^{j,\bar{z}}\right\|^2$$

$$\leq\left\|w_t-w_t^{j,\bar{z}}\right\|^2+\left(\eta_t^2-4\eta_t^2\left(1-\frac{\rho\eta_t}{2}\right)\right)\left\|\nabla\hat{f}_{z_{i_t},\bar{z}_{j_t}}(w_t)\nabla\hat{f}_{z_{i_t}\bar{z}_{j_t}}(w_t^{j,\bar{z}})\right\|^2+2\rho\eta_t\left\|w_{t+1}-w_{t+1}^{j,\bar{z}}\right\|^2$$

$$\leq\left\|w_t-w_t^{j,\bar{z}}\right\|^2+2\rho\eta_t\left\|w_{t+1}-w_{t+1}^{j,\bar{z}}\right\|^2,$$

where the second inequality is due to $\eta_t\leq\frac{1}{2\rho t}\leq\frac{3}{2\rho}$. The above inequality implies

$$\left\|w_{t+1}-w_{t+1}^{j,\bar{z}}\right\|\leq\frac{1}{\sqrt{1-2\rho\eta_t}}\left\|w_t-w_t^{j,\bar{z}}\right\|,$$

Secondly, when $j_t=j$, there holds

$$\left\|w_{t+1}-w_{t+1}^{j,\bar{z}}\right\|$$

$$=\left\|w_t-w_t^{j,\bar{z}}-\eta_t\tilde{\nabla}g_{z_{i_t}}(w_t)\tilde{\nabla}f_{\bar{z}_{j_t}}(v_{t+1})+\eta_t\tilde{\nabla}g_{z_{i_t}}(w_t^{j,\bar{z}})\tilde{\nabla}f_{\bar{z}_{j_t}'}(v_{t+1}^{j,\bar{z}})\right\|$$

$$\leq\left\|w_t-w_t^{j,\bar{z}}\right\|+\eta_t\left\|\tilde{\nabla}g_{z_{i_t}}(w_t)\tilde{\nabla}f_{\bar{z}_{j_t}}(v_{t+1})-\tilde{\nabla}g_{z_{i_t}}(w_t^{j,\bar{z}})\tilde{\nabla}f_{\bar{z}_{j_t}'}(v_{t+1}^{j,\bar{z}})\right\|$$

$$\leq\left\|w_t-w_t^{j,\bar{z}}\right\|+2\eta_t\left\|\tilde{\nabla}g_{z_{i_t}}(w_t)\right\|\left\|\tilde{\nabla}f_{\bar{z}_{j_t}}(v_{t+1})\right\|$$

$$\le \left\|w_t - w_t^{j,\bar{z}}\right\| + \frac{2M_f M_g}{\mu^2}\eta_t.$$

Combining the above two cases, we can get that

$$\left\|w_{t+1} - w_{t+1}^{j,\bar{z}}\right\| \le \frac{1}{\sqrt{1-2\rho\eta_t}}\left\|w_t - w_t^{j,\bar{z}}\right\|\mathbb{I}[j_t \neq j] + \left(\left\|w_t - w_t^{j,\bar{z}}\right\| + \frac{2M_f M_g}{\mu^2}\eta_t\right)\mathbb{I}[j_t = j].$$

Taking expectation over $i_t$,

$$\mathbb{E}_{i_t}\left[\left\|w_{t+1} - w_{t+1}^{j,\bar{z}}\right\|\right]$$

$$\le \frac{1}{\sqrt{1-2\rho\eta_t}}\left\|w_t - w_t^{j,\bar{z}}\right\|\mathbb{E}_{i_t}\left[\mathbb{I}[j_t \neq j]\right] + \left(\left\|w_t - w_t^{j,\bar{z}}\right\| + \frac{2M_f M_g}{\mu^2}\eta_t\right)\mathbb{E}_{i_t}\left[\mathbb{I}[j_t = j]\right]$$

$$\le \frac{1}{\sqrt{1-2\rho\eta_t}}\left\|w_t - w_t^{j,\bar{z}}\right\| + \frac{2M_f M_g}{\mu^2 m}\eta_t.$$

Then, taking expectation over $A$ and taking summation from $t = 1$ to $T-1$ to get that

$$\mathbb{E}_A\left[\left\|w_T - w_T^{j,\bar{z}}\right\|\right] \le \frac{1}{\sqrt{1-2\rho\eta_t}}\mathbb{E}_A\left[\left\|w_{T-1} - w_{T-1}^{j,\bar{z}}\right\|\right] + \frac{2M_f M_g}{\mu^2 m}\eta_t$$

$$\le \sum_{t=1}^{T-1}\left(\prod_{t'=t+1}^{T-1}\frac{1}{\sqrt{1-2\rho\eta_{t'}}}\right)\frac{2M_f M_g}{\mu^2 m}\eta_t$$

$$\le \sum_{t=1}^{T-1}\left(\prod_{t'=t+1}^{T-1}\sqrt{1+\frac{1}{t'-1}}\right)\frac{M_f M_g}{\rho\mu^2 m}\frac{1}{t}$$

$$\le \sum_{t=1}^{T-1}\left(\prod_{t'=2}^{T-1}\sqrt{1+\frac{1}{t'-1}}\right)\frac{M_f M_g}{\rho\mu^2 m}\frac{1}{t}$$

$$\le \sqrt{\prod_{t'=2}^{T-1}\exp\left\{\frac{1}{t'-1}\right\}}\sum_{t=1}^{T-1}\frac{M_f M_g}{\rho\mu^2 m}\frac{1}{t}$$

$$\le \sqrt{\exp\left\{\sum_{t'=2}^{T-1}\frac{1}{t'-1}\right\}}\frac{M_f M_g}{\rho\mu^2 m}\sum_{t=1}^{T-1}\frac{1}{t}$$

$$\le \frac{M_f M_g(eT)^{\frac{1}{2}}\log(eT)}{\rho\mu^2 m}, \tag{34}$$

where the fourth inequality is from $e^x \ge 1 + x$.

As for the optimization analysis of the full black-box SCGD, we can combine the proofs of Theorem 4 and Corollary 1 to get that

$$\mathbb{E}\left[F_S(w_T) - F_S(w(S))\right] \le \mathcal{O}\left(\mu^4 + d_2^2 + b^{-1}d_2\right), \tag{35}$$

where $d_2 = d - 2\sqrt{\left(p+\frac{1}{2}\right)\beta} + \left(p+\frac{1}{2}\right)\beta$. Combining Theorem 1, Equations (32), (34) and (35), we can get that

$$\mathbb{E}\left[F(w_T) - F(w^*)\right] \le \mathcal{O}\left(\left(n^{-1} + m^{-1}\right)T^{\frac{1}{2}}\log T + n^{-\frac{1}{2}} + \mu^4 + b^{-2}d_2^2 + b^{-1}d_2\right).$$

**SCSC:** Similar to the proof of SCGD except for

$$\nabla^2 \hat{f}_{z_{i_t},\bar{z}_{j_t}}(w_t)$$

$$= \frac{1}{b^2}\sum_{l=1}^{b}\frac{u_{t,l}}{\mu}\nabla g_{z_{i_t}}(w_t)\left(\nabla f_{\bar{z}_{j_t}}(v_{t+1} + \mu u_{t,l}) - \nabla f_{\bar{z}_{j_t}}(v_{t+1})\right)\sum_{l=1}^{b}\frac{u_{t,l}}{\mu}\left(g_{z_{i_t}}(w_t + \mu u_{t,l}) - g_{z_{i_t}}(w_t)\right)$$

$$+ \frac{1}{b^2}\sum_{l=1}^{b}\frac{u_{t,l}}{\mu}\left(f_{\bar{z}_{j_t}}(v_{t+1} + \mu u_{t,l}) - f_{\bar{z}_{j_t}}(v_{t+1})\right)\sum_{l=1}^{b}\frac{u_{t,l}}{\mu}\left(\nabla g_{z_{i_t}}(w_t + \mu u_{t,l}) - \nabla g_{z_{i_t}}(w_t)\right).$$

based on the update of SCSC, we have that, for $\eta_t \leq \frac{1}{2\rho t} \leq \frac{3}{2\rho}$, $\rho = \frac{1}{\mu^2}\left(L_g M_f' M_g + M_f M_g'\right)$,

$$\mathbb{E}_A\left[\left\|w_T - w_T^{i,z}\right\|\right] \leq \frac{M_f M_g (eT)^{\frac{1}{2}}\log(eT)}{\rho \mu^2 n} \tag{36}$$

and

$$\mathbb{E}_A\left[\left\|w_T - w_T^{j,\bar{z}}\right\|\right] \leq \frac{M_f M_g (eT)^{\frac{1}{2}}\log(eT)}{\rho \mu^2 m}. \tag{37}$$

As for the optimization analysis of the full black-box SCGD, we can combine the proofs of Theorem 4 and Corollary 1 to get that

$$\mathbb{E}\left[F_S(w_T) - F_S(w(S))\right] \leq \mathcal{O}\left(\mu^4 + b^{-2}d_2^2 + b^{-1}d_2\right), \tag{38}$$

where $d_2 = d - 2\sqrt{\left(p + \frac{1}{2}\right)} + \left(p + \frac{1}{2}\right)$. Combining Theorem 1, Equations (36), (37) and (38), we can get that

$$\mathbb{E}\left[F(w_T) - F(w^*)\right] \leq \mathcal{O}\left(\left(n^{-1} + m^{-1}\right)T^{\frac{1}{2}}\log T + n^{-\frac{1}{2}} + \mu^4 + b^{-2}d_2^2 + b^{-1}d_2\right).$$

$\square$

## E Proofs of Applications

Before stating our remain proofs, it should be noted that there are a few differences between the setting of FOO-based VFL (VFL-CZOFO) and the one of SCGD (SCSC). First of all, we set $S = \{z_1, ..., z_n\}$ and $S^{i,z} = \{z_1, ..., z_{i-1}, z_i', z_{i+1}, ..., z_n\}$ according to the learning paradigm of VFL. Secondly, the update of the outer model (global model) for FOO-based VFL (VFL-CZOFO) is not based on the simple weighted summation of SCGD (SCSC). Luckily, these differences will not make a difference in our proofs.

**Proof of Corollary 4**:

**FOO-based VFL:** Considering the independence of all clients, we just prove the corresponding result of the $k$-th client for some $k \in [K]$. Firstly, when $i_t \neq i$, there holds

$$\left\|w_{t+1}^k - w_{t+1}^{i,k}\right\|^2$$
$$= \left\|w_t^k - w_t^{i,k} - \eta_t \nabla g(w_t^k)\nabla f(g(w_t^k)) + \eta_t \nabla g(w_t^{i,k})\nabla f(g(w_t^{i,k}))\right\|^2$$
$$= \left\|w_t^k - w_t^{i,k}\right\|^2 + \eta_t^2 \left\|\nabla g(w_t^k)\nabla f(g(w_t^k)) - \nabla g(w_t^{i,k})\nabla f(g(w_t^{i,k}))\right\|^2$$
$$- 2\eta_t \left\langle w_t^k - w_t^{i,k}, \nabla g(w_t^k)\nabla f(g(w_t^k)) - \nabla g(w_t^{i,k})\nabla f(g(w_t^{i,k}))\right\rangle,$$

where $w_t^{i,z,k}$ is simplified as $w_t^{i,k}$. With Assumption 3, the terms $\nabla g(w_t^k)$ and $\nabla f(g(w_t^k))$ are both differentiable. Thus, $\nabla g(w_t^k)\nabla f(g(w_t^k))$ is also differentiable, which means that it is continuous on its domain. As we all know, a continuous function has primitive functions. Then, it is reasonable to assume that there exists a primitive function $\hat{f}(w_t^k)$ at least whose derivative function $\nabla \hat{f}(w_t^k) = \nabla g(w_t^k)\nabla f(g(w_t^k))$. For example, considering the independence between $v_t$ and $g(w_t^k)$, $f(g(w_t^k))$ is one of the primitive functions $\hat{f}(w_t^k)$. Then

$$\left\|w_{t+1}^k - w_{t+1}^{i,k}\right\|^2$$
$$= \left\|w_t^k - w_t^{i,k}\right\|^2 + \eta_t^2 \left\|\nabla \hat{f}(w_t^k)\nabla \hat{f}(w_t^{i,k})\right\|^2 - 2\eta_t \left\langle w_t^k - w_t^{i,k}, \nabla \hat{f}(w_t^k) - \nabla \hat{f}(w_t^{i,k})\right\rangle. \tag{39}$$

Taking derivative of $\nabla \hat{f}(w_t^k)$ over $w_t^k$, we get that

$$\nabla^2 \hat{f}(w_t^k) = \nabla^2 g(w_t^k)\nabla f(g(w_t^k)) + \left(\nabla g(w_t^k)\right)^2 \nabla^2 f(g(w_t^k)).$$

Thus,
$$\left\|\nabla^2 \hat{f}_{z_{i_t}, \bar{z}_{j_t}}(w_t)\right\| \le \alpha_g L_f + L_g^2 \alpha_f.$$

Let $\rho = \alpha_g L_f + L_g^2 \alpha_f$, then $\hat{f}(w_t^k)$ is $\rho$-smooth. And we can know that $\lambda_{\min}\left(\nabla^2 \hat{f}(w_t^k)\right) \ge -\left\|\nabla^2 \hat{f}(w_t^k)\right\| \ge -\rho$. According to Lemma 3, we can get that

$$\left\langle w_t^k - w_t^{i,k}, \nabla \hat{f}(w_t^k) - \nabla \hat{f}(w_t^{i,k})\right\rangle$$
$$\ge 2\eta_t \left(1 - \frac{\rho \eta_t}{2}\right) \left\|\nabla \hat{f}(w_t^k) - \nabla \hat{f}(w_t^{i,k})\right\|^2 - \rho \left\|w_t^k - w_t^{i,k} - \nabla \hat{f}(w_t^k) + \nabla \hat{f}(w_t^{i,k})\right\|^2$$
$$= 2\eta_t \left(1 - \frac{\rho \eta_t}{2}\right) \left\|\nabla \hat{f}(w_t^k) - \nabla \hat{f}(w_t^{i,k})\right\|^2 - \rho \left\|w_{t+1}^k - w_{t+1}^{i,k}\right\|^2.$$

Now, plugging the above inequality back into Equation (39) yields

$$\left\|w_{t+1}^k - w_{t+1}^{i,k}\right\|^2$$
$$\le \left\|w_t^k - w_t^{i,k}\right\|^2 + \left(\eta_t^2 - 4\eta_t^2\left(1 - \frac{\rho\eta_t}{2}\right)\right)\left\|\nabla\hat{f}(w_t^k)\nabla\hat{f}(w_t^{i,k})\right\|^2 + 2\rho\eta_t\left\|w_{t+1}^k - w_{t+1}^{i,k}\right\|^2$$
$$\le \left\|w_t^k - w_t^{i,k}\right\|^2 + 2\rho\eta_t\left\|w_{t+1}^k - w_{t+1}^{i,k}\right\|^2,$$

where the second inequality is due to $\eta_t \le \frac{1}{2\rho t} \le \frac{3}{2\rho}$. The above inequality implies

$$\left\|w_{t+1}^k - w_{t+1}^{i,k}\right\| \le \frac{1}{\sqrt{1 - 2\rho\eta_t}}\left\|w_t^k - w_t^{i,k}\right\|,$$

Secondly, when $i_t = i$, there holds

$$\left\|w_{t+1}^k - w_{t+1}^{i,k}\right\|$$
$$= \left\|w_t^k - w_t^{i,k} - \eta_t \nabla g(w_t^k)\nabla f(g(w_t^k)) + \eta_t \nabla g(w_t^{i,k})\nabla f(g(w_t^{i,k}))\right\|$$
$$\le \left\|w_t^k - w_t^{i,k}\right\| + \eta_t \left\|\nabla g(w_t^k)\nabla f(g(w_t^k)) - \nabla g(w_t^{i,k})\nabla f(g(w_t^{i,k}))\right\|$$
$$\le \left\|w_t^k - w_t^{i,k}\right\| + 2\eta_t \left\|\nabla g(w_t^k)\right\| \left\|\nabla f(g(w_t^k))\right\|$$
$$\le \left\|w_t^k - w_t^{i,k}\right\| + 2L_g L_f \eta_t.$$

Combining the above two cases, we can get that

$$\left\|w_{t+1}^k - w_{t+1}^{i,k}\right\| \le \frac{1}{\sqrt{1 - 2\rho\eta_t}}\left\|w_t^k - w_t^{i,k}\right\| \mathbb{I}[i_t \ne i] + \left(\left\|w_t^k - w_t^{i,k}\right\| + 2L_g L_f \eta_t\right)\mathbb{I}[i_t = i].$$

Taking expectation over $i_t$,

$$\mathbb{E}_{i_t}\left[\left\|w_{t+1}^k - w_{t+1}^{i,k}\right\|\right]$$
$$\le \frac{1}{\sqrt{1 - 2\rho\eta_t}}\left\|w_t^k - w_t^{i,k}\right\| \mathbb{E}_{i_t}\left[\mathbb{I}[i_t \ne i]\right] + \left(\left\|w_t^k - w_t^{i,k}\right\| + 2L_g L_f \eta_t\right)\mathbb{E}_{i_t}\left[\mathbb{I}[i_t = i]\right]$$
$$\le \frac{1}{\sqrt{1 - 2\rho\eta_t}}\left\|w_t^k - w_t^{i,k}\right\| + \frac{2L_g L_f}{n}\eta_t.$$

Then, taking expectation over $A$ and taking summation from $t = 1$ to $T - 1$ to get that

$$\mathbb{E}_A\left[\left\|w_T^k - w_T^{i,k}\right\|\right] \le \frac{1}{\sqrt{1 - 2\rho\eta_t}}\mathbb{E}_A\left[\left\|w_{T-1}^k - w_{T-1}^{i,k}\right\|\right] + \frac{2L_g L_f}{n}\eta_t$$
$$\le \sum_{t=1}^{T-1}\left(\prod_{t'=t+1}^{T-1}\frac{1}{\sqrt{1 - 2\rho\eta_{t'}}}\right)\frac{2L_g L_f}{n}\eta_t$$

$$\leq \sum_{t=1}^{T-1} \left( \prod_{t'=t+1}^{T-1} \sqrt{1 + \frac{1}{t'-1}} \right) \frac{L_g L_f}{\rho n} \frac{1}{t}$$

$$\leq \sum_{t=1}^{T-1} \left( \prod_{t'=2}^{T-1} \sqrt{1 + \frac{1}{t'-1}} \right) \frac{L_g L_f}{\rho n} \frac{1}{t}$$

$$\leq \sqrt{\prod_{t'=2}^{T-1} \exp \left\{ \frac{1}{t'-1} \right\}} \sum_{t=1}^{T-1} \frac{L_g L_f}{\rho n} \frac{1}{t}$$

$$\leq \sqrt{\exp \left\{ \sum_{t'=2}^{T-1} \frac{1}{t'-1} \right\}} \frac{L_g L_f}{\rho n} \sum_{t=1}^{T-1} \frac{1}{t}$$

$$\leq \frac{L_g L_f (eT)^{\frac{1}{2}} \log(eT)}{\rho n}, \tag{40}$$

where the fourth inequality is from $e^x \geq 1 + x$. Combining Theorem 3 and Equation (40), we can get that

$$\mathbb{E} \left[ |F(w_T^k) - F_S(w_T^k)| \right] \leq \mathcal{O} \left( n^{-1} T^{\frac{1}{2}} \log T \right).$$

$\square$

**Proof of Corollary 5**: Similar to the proofs of Theorem 4 and Corollary 4, we can get the result of Corollary 5. $\square$

## F  Key Challenges and Technical Tools

In this section, the key challenges and technical tools of extending the theoretical analysis for SCO problems from white-box cases to black-box cases are listed as follows.

**(1) Generalization:** Considering three different types of black-box SCO methods, we apply our new non-convex analysis (Theorem 3) to these cases (Theorem 4, Corollary 1 and 2) in Section 3.2. Due to the difference related to function form, there are some differences related to the upper bounds of first-order and second-order gradients of $\tilde{\nabla} f$ between Theorem 3 and the generalization part of Theorem 4. The differences among Theorem 3 and Corollary 1, Corollary 2 are the same as Theorem 4.

**(2) Optimization:** For optimization, the estimated gradient does introduce several extra terms regarding the accuracy of the gradient estimation, i.e., $\tilde{\nabla} f - (p + 1/2)\beta \nabla f$ and $\tilde{\nabla} f - \beta \nabla f$. These terms are derived from some special strategies (such as a special decomposition $\tilde{\nabla} f = \tilde{\nabla} f + (p + 1/2)\beta \nabla f - (p + 1/2)\beta \nabla f$). We propose an extended lemma (Lemma 6) from [39] and combine this lemma with these strategies to limit the expansion of $\mathbb{E}[F_S(w_{t+1}) - F_S(w(S))]$ during the iterations. Otherwise, these extra terms will lead to the divergence of our result.

Finally, we want to emphasize our advantages compared with previous work related to the generalization guarantee of SCO [21].

**(1) Better results:** For convex optimization, Theorem 2 leverages the co-coercivity property of convex and smooth function to provide the stability bound $\mathcal{O}((n^{-1} + m^{-1})\beta \log T)$ under milder parameter selection than [21]. And our proof is more concise since it avoids the intermediate step which measures the distance between $v$ and $g(w)$ in the analysis of [21].

**(2) Non-convex guarantee:** We leverage a special lemma, almost co-coercivity lemma, to develop our proof framework to non-convex case to obtain the first stability bound $\mathcal{O}((n^{-1} + m^{-1})T^{\frac{1}{2}} \log T)$ under milder parameter selection than [39].

